# Rethinking the Diffusion Models for Missing Data Imputation: A Gradient Flow Perspective

**Zhichao Chen**[1]   **Haoxuan Li**[2]   **Fangyikang Wang**[1]   **Odin Zhang**[3]   **Hu Xu**[1]
**Xiaoyu Jiang**[1]   **Zhihuan Song**[1,4]   **Hao Wang**[1*]
[1]Zhejiang University   [2]Peking University   [3]University of Washington
[4]Guangdong University of Petrochemical Technology
12032042@zju.edu.cn   hxli@stu.pku.edu.cn   wangfangyikang@zju.edu.cn
odinz@uw.edu   hxu_zju@zju.edu.cn   jiangxiaoyu@zju.edu.cn
songzhihuan@zju.edu.cn   haohaow@zju.edu.cn

## Abstract

Diffusion models have demonstrated competitive performance in missing data imputation (MDI) task. However, directly applying diffusion models to MDI produces suboptimal performance due to two primary defects. First, the sample diversity promoted by diffusion models hinders the accurate inference of missing values. Second, data masking reduces observable indices for model training, obstructing imputation performance. To address these challenges, we introduce Negative Entropy-regularized Wasserstein gradient flow for Imputation (NewImp), enhancing diffusion models for MDI from a gradient flow perspective. To handle the first defect, we incorporate a negative entropy regularization term into the cost functional to suppress diversity and improve accuracy. To handle the second defect, we demonstrate that the imputation procedure of NewImp, induced by the conditional distribution-related cost functional, can equivalently be replaced by that induced by the joint distribution, thereby naturally eliminating the need for data masking. Extensive experiments validate the effectiveness of our method. Code is available at https://github.com/JustusvLiebig/NewImp.

## 1 Introduction

Missing data is a pervasive problem for data analytics in diverse scenarios, including e-commerce [29, 30, 57], healthcare [51, 56], and process industry [33, 58]. For instance, in healthcare, patient monitoring devices may fail or lose connection, leading to missing vital signs data. Similarly, in industrial processes, sensor signals may be incomplete due to inevitable mechanical shock. These incompletenesses hamper data integrity and impede subsequent analysis. Therefore, accurate missing data imputation (MDI) is critical for enabling reliable analysis and decision in real-world applications.

Recently, diffusion models (DMs) have emerged as a powerful tool for MDI [66]. Specifically, these models first estimate the (Stein) score function of the missing data conditioned on the observed data, subsequently reformulating the imputation problem as a generative task grounded in the learned score function. These works are initiated from [51] and evolve to incorporate crafted model architecture [33] and learning objectives [38, 73] for enhancing the accuracy of score estimation [41]. Celebrated for their advantageous capability to model data distributions and generate high-quality synthetic data [41, 50, 65], diffusion models have been a prevalent approach to MDI.

Despite the successes of diffusion models, we argue that directly applying diffusion models to MDI results in suboptimal performance due to two primary limitations. First, diffusion models perform

---

imputation by sampling from a learned score function, which inadvertently promotes diversity in the imputed values. This increased diversity contradicts the accuracy required for precise imputation of missing data [38]. Second, the training process involves masking a portion of the observed data as labels. The selection of masking strategy significantly impacts imputation accuracy and is inherently challenging to optimize [51]. Moreover, the masked data during training often differ in missing mechanisms from those encountered during testing, resulting in a discrepancy between training and inference phases that degrades performance. Consequently, diffusion models introduce unintended diversity and impose data masking, both of which impede effective imputation.

To tackle these challenges, we introduce a novel DM-based MDI approach termed Negative Entropy-regularized Wasserstein Gradient Flow Imputation (NewImp). Specifically, to handle the first issue, we revisit DM-based MDI task within the Wasserstein Gradient Flow (WGF) framework, derive the associated cost functionals, and identify that they implicitly promote diversity in the imputed values. Building on this insight, we incorporate a negative entropy-regularized (NER) cost functional to suppress imputation diversity and enhance accuracy. Furthermore, we derive a closed-form imputation procedure based on the proposed cost functional within the reproducing kernel Hilbert space (RKHS). After that, we further prove that within the WGF framework, the imputation procedure of NewImp, induced from the cost functional associated with conditional distribution, can be induced from another cost functional associated with joint distribution equivalently, within which we merely need to estimate the joint distribution during the model training stage, thereby naturally eliminating the need for data masking.

**Contributions.** The main contributions of this paper are summarized as follows:

- We demonstrate that directly applying diffusion models to MDI causes suboptimal performance, as they prompt unintended diversity and require data masking, both impeding accurate imputation.

- We propose NewImp, a novel DM-based MDI approach under the WGF framework which introduces an NER cost functional to suppress unintended diversity. Based on this, we further prove that the imputation procedure of NewImp can be induced from an equivalent joint-distribution-related functional, and consequently introduce an imputation procedure that sidesteps the data masking.

- We conduct various experiments over public numerical tabular datasets to demonstrate the superiority of the NewImp method over prevalent baseline models.

## 2 Preliminaries

### 2.1 Problem Formulation

Suppose $\boldsymbol{X}^{(\text{ideal})} \in \mathbb{R}^{N \times D}$ represents an ideal numerical tabular dataset without any missing entries, where N and D denote the number of samples and features, respectively. The observed dataset is expressed as: $\boldsymbol{X}^{(\text{obs})} = \boldsymbol{X}^{(\text{ideal})} \odot \boldsymbol{M} + \text{NaN} \odot (\mathbb{1}_{N \times D} - \boldsymbol{M})$, where $\odot$ denotes the Hadamard product, $\mathbb{1}_{N \times D}$ is a matrix of ones of size $N \times D$, and $\boldsymbol{M} \in \{0, 1\}^{N \times D}$ is a binary mask that indicates the presence (1) or absence (0) of data in each entry. The task of MDI involves imputing the missing entries in $\boldsymbol{X}^{(\text{obs})}$. This is achieved by constructing a matrix $\hat{\boldsymbol{X}} = \boldsymbol{X}^{(\text{obs})} \odot \boldsymbol{M} + \boldsymbol{X}^{(\text{imp})} \odot (\mathbb{1}_{N \times D} - \boldsymbol{M})$, where $\boldsymbol{X}^{(\text{imp})}$ is the matrix containing the imputed values.

The missing mechanism can be classified into three categories [44]: Missing Completely at Random (MCAR), Missing at Random (MAR), and Missing Not at Random (MNAR) (Detailed information about missing mechanisms is given in Appendix E.1). Notably, in the MNAR setting, it is generally difficult to identify the missing data distribution without additional assumptions and constraints [22]. Hence, our discussion primarily focuses on numerical tabular data with MAR and MCAR settings.

### 2.2 Diffusion Models and Its Application for MDI Task

Diffusion models function by gradually corrupting data towards a tractable noise distribution, such as a standard Gaussian, and subsequently reversing this corruption to generate samples [50]. Specifically, the forward corruption process is modeled as a discretization of a stochastic differential equation (SDE) over time $\tau$: $d\boldsymbol{X}_\tau = f(\boldsymbol{X}_\tau)d\tau + g_\tau dW_\tau$, where $f(\boldsymbol{X}_\tau)$ is drift term, $g_\tau$ is volatility term, and $dW\tau$ is standard Wiener process. The solution to this SDE creates a continuous trajectory of random variables $\boldsymbol{X}_\tau|_{\tau=0}^{\text{T}}$. The density function $q_\tau$ of $\boldsymbol{X}_\tau$ adheres to the Fokker-Planck-Kolmogorov (FPK)

equation: $\frac{\partial q_\tau}{\partial \tau} = -\nabla \cdot (q_\tau f(\boldsymbol{X}\tau)) + \frac{1}{2}g_\tau^2 \nabla \cdot \nabla q_\tau$ (see Theorem 5.4 in reference [47]). The reverse process is governed by: $\mathrm{d}\boldsymbol{X}_\tau = [f(\boldsymbol{X}_\tau) - g_\tau^2 \nabla \log p(\boldsymbol{X}_\tau)]\mathrm{d}\tau + g_\tau \mathrm{d}W_\tau$ [3], where $\nabla \log p(\boldsymbol{X}_\tau)$ represents the score function, which is often parameterized by neural networks.

Diffusion models treat MDI as a conditional generation task. The score function, $\nabla \log p(\boldsymbol{X})$, is defined specifically for MDI as $\nabla_{\boldsymbol{X}^{(\mathrm{miss})}} \log p(\boldsymbol{X}^{(\mathrm{miss})}|\boldsymbol{X}^{(\mathrm{obs})})$ [51], and the MDI task is executed by generating samples based on this conditional score function. The key challenge is to obtain an estimation $\nabla_{\boldsymbol{X}^{(\mathrm{miss})}} \log \hat{p}(\boldsymbol{X}^{(\mathrm{miss})}|\boldsymbol{X}^{(\mathrm{obs})})$ that approximates $\nabla_{\boldsymbol{X}^{(\mathrm{miss})}} \log p(\boldsymbol{X}^{(\mathrm{miss})}|\boldsymbol{X}^{(\mathrm{obs})})$. Given that the true $\boldsymbol{X}^{(\mathrm{miss})}$ is unknown, existing DM-based approaches utilize a mask matrix to drop some observable data as labels. However, the specification of the mask mechanism, determining the effectiveness of $\nabla \log \hat{p}(\boldsymbol{X}^{(\mathrm{miss})}|\boldsymbol{X}^{(\mathrm{obs})})$, is challenging since it should align with the data missing mechanism in the testing dataset [51], which may be unknown in practice.

## 2.3 Wasserstein Gradient Flow

Wasserstein space $\mathcal{P}_2(\mathbb{R}^D)$ is defined as the set of distributions with finite second-order moments. Consider a cost functional $\mathcal{F}_{\mathrm{cost}} : \mathcal{P}_2(\mathbb{R}^D) \to \mathbb{R}$; the celebrated Wasserstein gradient flow (WGF) is an absolutely continuous trajectory $(q_\tau)_{\tau>0}$ in this space, which evolves over time $\tau$ to minimize $\mathcal{F}_{\mathrm{cost}}$ efficiently. This dynamic is governed by the continuity equation:

$$\frac{\partial q_\tau}{\partial \tau} = -\nabla \cdot (u_\tau q_\tau), \quad u_\tau = -\nabla_{\boldsymbol{X}} \frac{\delta \mathcal{F}_{\mathrm{cost}}}{\delta q_\tau} \tag{1}$$

where $u_\tau : \mathbb{R}^D \to \mathbb{R}^D$ is a time-dependent *velocity field* [2], whose input is sample $\boldsymbol{X} \in \mathbb{R}^D$; $\frac{\delta \mathcal{F}_{\mathrm{cost}}}{\delta q_\tau}$ denotes the first variation of $\mathcal{F}_{\mathrm{cost}}$ with respect to $q_\tau$. On this basis, the evolution of $\boldsymbol{X}$ over time $\tau$ in $\mathcal{P}_2(\mathbb{R}^D)$ can be modeled by the ordinary differential equation (ODE):

$$\frac{\mathrm{d}\boldsymbol{X}}{\mathrm{d}\tau} = u_\tau \tag{2}$$

However, simulating this ODE is challenging since $u_\tau$ involves the estimation of $q_\tau$, which involves solving the differential equation $\frac{\partial q_\tau}{\partial \tau} = -\nabla \cdot (u_\tau q_\tau)$ that proves to be not analytically solvable [16].

# 3 Motivations

## 3.1 Diffusion Models Secretly Foster Diversity

Based on the notations defined in Section 2.1, we can first define the following cost functional for MDI task according to the maximum likelihood estimation principle, where we want to find the value with the highest probability:

$$\boldsymbol{X}^{(\mathrm{imp})} = \underset{\boldsymbol{X}^{(\mathrm{miss})}}{\arg\max} \log \hat{p}(\boldsymbol{X}^{(\mathrm{miss})}|\boldsymbol{X}^{(\mathrm{obs})}), \tag{3}$$

where $\hat{p}(\boldsymbol{X}^{(\mathrm{miss})}|\boldsymbol{X}^{(\mathrm{obs})})$ is the estimation of $p(\boldsymbol{X}^{(\mathrm{miss})}|\boldsymbol{X}^{(\mathrm{obs})})$ via neural network [52]. Notably, we can treat $\boldsymbol{X}^{(\mathrm{miss})}$ as samples from a 'proposal distribution' $r(\boldsymbol{X}^{(\mathrm{miss})})$, and formulate the following optimization problem based on variational inference [27, 70]:

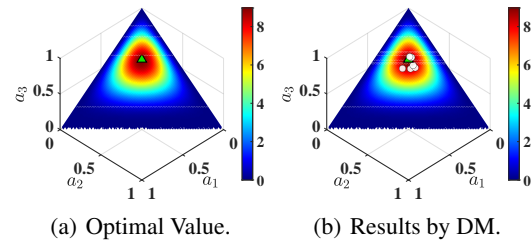

(a) Optimal Value.     (b) Results by DM.

Figure 1: Comparison of the optimal point in green triangle and the results obtained by diffusion models in white scatters. See details in Appendix B.

$$\underset{r(\boldsymbol{X}^{(\mathrm{miss})})}{\arg\max} \quad \mathbb{E}_{r(\boldsymbol{X}^{(\mathrm{miss})})}[\log \hat{p}(\boldsymbol{X}^{(\mathrm{miss})}|\boldsymbol{X}^{(\mathrm{obs})})], \tag{4}$$

where we aim to sample some $\boldsymbol{X}^{(\mathrm{miss})}$ samples from proposal distribution $r(\boldsymbol{X}^{(\mathrm{miss})})$, realize the maximum log-likelihood estimation over the sampled results, and 'optimize' the proposal distribution $r(\boldsymbol{X}^{(\mathrm{miss})})$ that is represented by samples $\boldsymbol{X}^{(\mathrm{miss})}$. Notably, in Eq. (4), we use the spirit from previous references represented by [32], where optimizing the samples $\boldsymbol{X}^{(\mathrm{miss})}$ is equivalent to optimizing the distribution $r(\boldsymbol{X}^{(\mathrm{miss})})$.

Referring to Eq. (4), it is observed that the MDI task can be formulated as an optimization problem. This prompts a pertinent question: If the conditional distribution $\log p(\boldsymbol{X}_i^{(\text{miss})}|\boldsymbol{X}_i^{(\text{obs})})$ is estimated accurately, *what would happen if we directly apply diffusion models to solve the optimization problem corresponding to MDI task?* To explore this, we consider a hypothetical scenario: Suppose we are optimizing a cost functional related to a three-dimensional Dirichlet distribution on the simplex $\Delta^2$:

$$\arg\max_{\boldsymbol{a}_h \in \Delta^2} \sum_{h=1}^{\text{H}} \{\log \frac{\Gamma(\sum_{k=1}^3 \rho_k)}{\prod_{k=1}^3 \Gamma(\rho_k)} + \sum_{k=1}^3 (\rho_k - 1) \log \boldsymbol{a}_{k,h}\}, \text{H}=8, \rho|_{k=1}^3 = [2.5, 2.5, 5.0],$$

where $\boldsymbol{a}_h|_1^{\text{H}}$ are variables, H is variable number, $\rho_k|_{k=1}^3$ is concentration parameter, and $\Gamma(\cdot)$ is gamma function. We compare the analytically derived optimal value with the results from diffusion models in Fig. 1. The diffusion model's results tend to surround but do not exactly reach the optimal value, suggesting that there might be *implicit, diversity-encouraging terms* integrated into the diffusion models' objectives that produce the observed inaccuracies. Identifying and modifying these regularization terms is crucial for enhancing the efficacy of diffusion models for MDI tasks.

## 3.2 Negative Entropy Regularization Term for Diversity Suppression

In this section, we identify and refine the terms in diffusion models' objectives that prompt unintended diversity and impede accurate imputation. We observe that the inference process in diffusion models adheres to the FPK equation, which is a specialized form of the continuity equation in WGF (see Sections 2.2 and 2.3). This alignment inspires us to reframe diffusion models within the WGF framework, enabling the derivation of their underlying cost functionals. By doing so, we can compare these functionals with the objective functional for MDI in Eq. (4)[2].

**Proposition 3.1.** *Within WGF framework, DM-based MDI approaches can be viewed as finding the imputed values $\boldsymbol{X}^{(imp)}$ that maximize the following objective:*

$$\arg\max_{r(\boldsymbol{X}^{(miss)})} \mathbb{E}_{r(\boldsymbol{X}^{(miss)})}[\log \hat{p}(\boldsymbol{X}^{(miss)}|\boldsymbol{X}^{(obs)})] + \psi(\boldsymbol{X}^{(miss)}) + const, \tag{5}$$

*where 'const' is the abbreviation of constant, and $\psi(\boldsymbol{X}^{(miss)})$ is a scalar function determined by the type of SDE underlying the diffusion models.*

- **VP-SDE:** $\psi(\boldsymbol{X}^{(miss)}) = \frac{1}{2}\mathbb{H}[r(\boldsymbol{X}^{(miss)})] + \mathbb{E}_{r(\boldsymbol{X}^{(miss)})}\{\frac{1}{4}[\boldsymbol{X}^{(miss)}]^\top[\boldsymbol{X}^{(miss)}]\} \geq 0$

- **VE-SDE:** $\psi(\boldsymbol{X}^{(miss)}) = \frac{1}{2}\mathbb{H}[r(\boldsymbol{X}^{(miss)})] \geq 0$

- **sub-VP-SDE:** $\psi(\boldsymbol{X}^{(miss)}) = \frac{1}{2}\mathbb{H}[r(\boldsymbol{X}^{(miss)})] + \mathbb{E}_{r(\boldsymbol{X}^{(miss)})}\{\frac{1}{4\gamma_\tau}[\boldsymbol{X}^{(miss)}]^\top[\boldsymbol{X}^{(miss)}]\} \geq 0,$

*where $\mathbb{H}[r(\boldsymbol{X}^{(miss)})] := -\int r(\boldsymbol{X}^{(miss)}) \log r(\boldsymbol{X}^{(miss)}) d\boldsymbol{X}^{(miss)}$ is the entropy term, $\gamma_\tau$ is determined by noise scale $\beta_\tau$: $\gamma_\tau := (1 - \exp(-2\int_0^\tau \beta_s ds)) > 0, 0 < \beta_1 < \cdots < \beta_{\text{T}} < 1.$*

Proposition 3.1 reveals that diffusion models inherently optimize an objective functional that largely aligns with (4), but they secretly include an additional term $\psi(\boldsymbol{X}^{(\text{miss})}) > 0$. This term makes Eq. (5) an *upper bound* on Eq. (4), i.e., maximizing the cost functional in Eq. (5) does not guarantee to maximize the MDI objective in Eq. (4). Furthermore, the entropy term included in the models fosters sample diversity, which may compromise the accuracy required in MDI tasks [53, 38]. To address this issue, we propose incorporating a negative entropy term as $\psi(\boldsymbol{X}^{(\text{miss})})$:

$$\psi(\boldsymbol{X}^{(\text{miss})}) = -\lambda \mathbb{H}[r(\boldsymbol{X}^{(\text{miss})})], \tag{6}$$

where $\lambda > 0$ is a predefined regularization strength, and consequently we can define our NER cost functional for MDI task as follows:

$$\mathcal{F}_{\text{NER}} := \mathbb{E}_{r(\boldsymbol{X}^{(\text{miss})})}[\log \hat{p}(\boldsymbol{X}^{(\text{miss})}|\boldsymbol{X}^{(\text{obs})})] - \lambda \mathbb{H}[r(\boldsymbol{X}^{(\text{miss})})]. \tag{7}$$

The objective functional in Eq. (7) provides a *lower bound* of Eq. (4). Therefore, maximizing $\mathcal{F}_{\text{NER}}$ guarantees maximizing Eq. (4). Meanwhile, $\mathcal{F}_{\text{NER}}$ effectively reduces the unintended diversity term, contributing to an improvement in imputation accuracy.

# 4 Implementation of the NewImp

## 4.1 Optimizing the $\mathcal{F}_{\text{NER}}$ within WGF Framework

In this section, we aim to optimize $\mathcal{F}_{\text{NER}}$ within the WGF framework [46, 69]. To this end, we plug Eq. (7) into Eq. (1), producing the velocity field below that drives the ODE in Eq. (2):

$$u(\boldsymbol{X}^{(\text{miss})}) = -\nabla_{\boldsymbol{X}^{(\text{miss})}} \frac{\delta(-\mathcal{F}_{\text{NER}})}{\delta r(\boldsymbol{X}^{(\text{miss})})} = [\nabla_{\boldsymbol{X}^{(\text{miss})}} \log \hat{p}(\boldsymbol{X}^{(\text{miss})}|\boldsymbol{X}^{(\text{obs})}) + \lambda \nabla_{\boldsymbol{X}^{(\text{miss})}} \log r(\boldsymbol{X}^{(\text{miss})})],$$

However, as stated in Section 2.3, implementing this ODE in computer code is intricate due to the intractability of the density function $r(\boldsymbol{X}^{(\text{miss})})$. Fortunately, by restricting the velocity field within the Reproducing Kernel Hilbert Space (RKHS) defined by the kernel function $u(\boldsymbol{X}^{(\text{miss})}) \in K(\boldsymbol{X}^{(\text{miss})}, \tilde{\boldsymbol{X}}^{(\text{miss})})$, an alternative ODE minimizing $\mathcal{F}_{\text{NER}}$ can be implemented in Proposition 4.1 [35, 31] which sidesteps the intractable $r(\boldsymbol{X}^{(\text{miss})})$[3].

**Proposition 4.1.** *Suppose $u(\boldsymbol{X}^{(miss)})$ is a velocity field regularized by the RKHS norm under the following conditions: 1). The kernel function satisfies: $\lim_{\|\boldsymbol{X}^{(miss)}\| \to \infty} K(\boldsymbol{X}^{(miss)}, \tilde{\boldsymbol{X}}^{(miss)}) = 0$. 2). The density $r(\boldsymbol{X}^{(miss)})$ is bounded. Then, the velocity field that minimizes the cost functional $\mathcal{F}_{NER} = \mathbb{E}_{r(\boldsymbol{X}^{(miss)})}[\log \hat{p}(\boldsymbol{X}^{(miss)}|\boldsymbol{X}^{(obs)})] - \lambda \mathbb{H}[r(\boldsymbol{X}^{(miss)})]$ can be given by:*

$$u(\boldsymbol{X}^{(miss)}) = \mathbb{E}_{r(\tilde{\boldsymbol{X}}^{(miss)})} \left\{ \begin{array}{l} -\lambda \nabla_{\tilde{\boldsymbol{X}}^{(miss)}} K(\boldsymbol{X}^{(miss)}, \tilde{\boldsymbol{X}}^{(miss)}) \\ + [\nabla_{\tilde{\boldsymbol{X}}^{(miss)}} \log \hat{p}(\tilde{\boldsymbol{X}}^{(miss)}|\boldsymbol{X}^{(obs)})]^\top K(\boldsymbol{X}^{(miss)}, \tilde{\boldsymbol{X}}^{(miss)}) \end{array} \right\}. \quad (8)$$

*where the expectation term $\mathbb{E}_{r(\tilde{\boldsymbol{X}}^{(miss)})}$ can be efficiently estimated using Monte Carlo approximation, $K(\boldsymbol{X}, \tilde{\boldsymbol{X}})$ is set as the radial basis function (RBF) kernel.*

## 4.2 Sidestepping Mask Matrix: Conditional Modeling via Joint Modeling

Simulating the ODE in Eq. (2) with Eq. (8) necessitates an accurate estimation of $p(\boldsymbol{X}^{(\text{miss})}|\boldsymbol{X}^{(\text{obs})})$. However, this modeling is challenging due to the diverse choices of masking matrices. More specifically, the accuracy of the estimated conditional distribution $p(\boldsymbol{X}^{(\text{miss})}|\boldsymbol{X}^{(\text{obs})})$ heavily relies on the selection of these matrices, and these matrices should be consistent with the data missing mechanism in the testing dataset, which may be unknown in practice [51]. To bypass this difficulty, we suggest substituting the conditional distribution $p(\boldsymbol{X}^{(\text{miss})}|\boldsymbol{X}^{(\text{obs})})$ with the joint distribution $p(\boldsymbol{X}^{(\text{joint})})$, where $\boldsymbol{X}^{(\text{joint})} = (\boldsymbol{X}^{(\text{miss})}, \boldsymbol{X}^{(\text{obs})})$. Building on this substitution, the velocity field is redefined based on the estimated joint distribution $\hat{p}(\boldsymbol{X}^{(\text{joint})})$ as follows:

$$u(\boldsymbol{X}^{(\text{joint})}) = \mathbb{E}_{r(\tilde{\boldsymbol{X}}^{(\text{joint})})} \left\{ \begin{array}{l} -\lambda \nabla_{\tilde{\boldsymbol{X}}^{(\text{miss})}} K(\boldsymbol{X}^{(\text{joint})}, \tilde{\boldsymbol{X}}^{(\text{joint})}) \\ + [\nabla_{\tilde{\boldsymbol{X}}^{(\text{miss})}} \log \hat{p}(\tilde{\boldsymbol{X}}^{(\text{joint})})]^\top K(\boldsymbol{X}^{(\text{joint})}, \tilde{\boldsymbol{X}}^{(\text{joint})}) \end{array} \right\}, \quad (9)$$

where $\nabla_{\tilde{\boldsymbol{X}}^{(\text{miss})}} \log \hat{p}(\tilde{\boldsymbol{X}}^{(\text{joint})})$ can be obtained by masking the $\nabla_{\boldsymbol{X}^{(\text{joint})}} \log \hat{p}(\boldsymbol{X}^{(\text{joint})})$ with the missing data indicator matrix $\boldsymbol{M}$ as follows:

$$\nabla_{\tilde{\boldsymbol{X}}^{(\text{miss})}} \log \hat{p}(\tilde{\boldsymbol{X}}^{(\text{joint})}) = \nabla_{\tilde{\boldsymbol{X}}^{(\text{joint})}} \log \hat{p}(\tilde{\boldsymbol{X}}^{(\text{joint})}) \odot (\mathbb{1}_{\text{N} \times \text{D}} - \boldsymbol{M}) + 0 \times \boldsymbol{M}, \quad (10)$$

and the expression of kernel function term can be *directly* given based on the expression of RBF:

$$K(\boldsymbol{X}^{(\text{joint})}, \tilde{\boldsymbol{X}}^{(\text{joint})}) = \exp(-\frac{\|\boldsymbol{X}^{(\text{joint})} - \tilde{\boldsymbol{X}}^{(\text{joint})}\|^2}{2h^2}), \quad (11)$$

where $h$ is the bandwidth, *the values of $\tilde{\boldsymbol{X}}^{(joint)}$ and $\boldsymbol{X}^{(joint)}$ are identical*, and the tilde notation on $\tilde{\boldsymbol{X}}^{(joint)}$ is merely used to *distinguish the variable with respect to which the derivative is taken*. On

this basis, similar to Eq. (10), the gradient term $\nabla_{\tilde{\boldsymbol{X}}^{(\text{miss})}} K(\boldsymbol{X}^{(\text{joint})}, \tilde{\boldsymbol{X}}^{(\text{joint})})$ can be given as follows:

$$\nabla_{\tilde{\boldsymbol{X}}^{(\text{miss})}} K(\boldsymbol{X}^{(\text{joint})}, \tilde{\boldsymbol{X}}^{(\text{joint})}) = \nabla_{\tilde{\boldsymbol{X}}^{(\text{joint})}} K(\boldsymbol{X}^{(\text{joint})}, \tilde{\boldsymbol{X}}^{(\text{joint})}) \odot (\mathbb{1}_{\text{N} \times \text{D}} - \boldsymbol{M}) + 0 \times \boldsymbol{M}, \quad (12)$$

and since $K(\boldsymbol{X}^{(\text{joint})}, \tilde{\boldsymbol{X}}^{(\text{joint})})$ is a smooth function, $\nabla_{\tilde{\boldsymbol{X}}^{(\text{joint})}} K(\boldsymbol{X}^{(\text{joint})}, \tilde{\boldsymbol{X}}^{(\text{joint})})$ can be *easily* computed by automatic-differentiation-based deep learning backends like by PyTorch [42].

Proposition 4.2 demonstrates that the cost functional $\mathcal{F}_{\text{joint-NER}}$, associated with Eq. (10), and $\mathcal{F}_{\text{NER}}$ exhibit a constant gap, indicating that optimizing $\mathcal{F}_{\text{joint-NER}}$ is equivalent to optimizing $\mathcal{F}_{\text{NER}}$.

**Proposition 4.2.** *Assume that the proposal distribution $r(\boldsymbol{X}^{(joint)})$ is factorized by $r(\boldsymbol{X}^{(joint)}) :=$ $r(\boldsymbol{X}^{(miss)})p(\boldsymbol{X}^{(obs)})$. The cost functional associated with the joint distribution is defined as follows:*

$$\mathcal{F}_{joint\text{-}NER} := \mathbb{E}_{r(\boldsymbol{X}^{(joint)})}[\log \hat{p}(\boldsymbol{X}^{(joint)})] - \lambda \mathbb{H}[r(\boldsymbol{X}^{(joint)})], \quad (13)$$

*which leads to the velocity field delineated in Eq. (9) and establishes $\mathcal{F}_{joint\text{-}NER}$ as a lower bound for $\mathcal{F}_{NER}$, with the difference being a constant (i.e., $\mathcal{F}_{joint\text{-}NER} = \mathcal{F}_{NER} - const, const \geq 0$).*

The detailed justification for the factorization $r(\boldsymbol{X}^{(\text{joint})}) := r(\boldsymbol{X}^{(\text{miss})})p(\boldsymbol{X}^{(\text{obs})})$ is provided in Appendix C. Based on this proposition, the following corollary can be obtained:

**Corollary 4.3.** *The following equation holds: $u(\boldsymbol{X}^{(joint)}) = u(\boldsymbol{X}^{(miss)})$.*

So far, we know that $u(\boldsymbol{X}^{(\text{joint})})$ can reduce $\mathcal{F}_{\text{NER}}$ as effectively as possible, which indicates that the velocity field defined in Eq. (9) can fully substitute for Eq. (8) in optimizing $\mathcal{F}_{\text{NER}}$ without loss of accuracy. Finally, the imputed value can be obtained by simulating the following ODE:

$$\frac{\mathrm{d}\boldsymbol{X}^{(\text{miss})}}{\mathrm{d}\tau} = u(\boldsymbol{X}^{(\text{joint})}). \quad (14)$$

### 4.3 Estimating the Joint Distribution

The remaining problem is to determine the estimation of score function $\nabla_{\boldsymbol{X}^{(\text{joint})}} \log \hat{p}(\boldsymbol{X}^{(\text{joint})})$. To achieve this, we employ Denoising Score Matching (DSM) [21, 52] to train the score function $\nabla_{\boldsymbol{X}^{(\text{joint})}} \log \hat{p}(\boldsymbol{X}^{(\text{joint})})$ parameterized by a neural network. Specifically, the learning objective is designed to minimize the discrepancy between the actual score and the model's predicted score after introducing Gaussian noise to the clean $\boldsymbol{X}^{(\text{joint})}$ as $\hat{\boldsymbol{X}}^{(\text{joint})}$:

$$\mathcal{L}_{\text{DSM}} := \frac{1}{2} \mathbb{E}_{q_\sigma(\hat{\boldsymbol{X}}^{(\text{joint})}|\boldsymbol{X}^{(\text{joint})})} [\|\nabla_{\hat{\boldsymbol{X}}^{(\text{joint})}} \log \hat{p}(\hat{\boldsymbol{X}}^{(\text{joint})}) - \nabla_{\hat{\boldsymbol{X}}^{(\text{joint})}} \log q_\sigma(\hat{\boldsymbol{X}}^{(\text{joint})}|\boldsymbol{X}^{(\text{joint})})\|^2]. \quad (15)$$

Notably, $\sigma$ is variance scale, $\hat{\boldsymbol{X}}^{(\text{joint})}$ is obtained by $\hat{\boldsymbol{X}}^{(\text{joint})} = \boldsymbol{X}^{(\text{joint})} + \epsilon, \epsilon \sim \mathcal{N}(\boldsymbol{0}, \sigma^2 \mathbf{I})$, and $\nabla_{\hat{\boldsymbol{X}}^{(\text{joint})}} \log q_\sigma(\hat{\boldsymbol{X}}^{(\text{joint})}|\boldsymbol{X}^{(\text{joint})}) = -\frac{\hat{\boldsymbol{X}}^{(\text{joint})} - \boldsymbol{X}^{(\text{joint})}}{\sigma^2}$. Once $\nabla_{\boldsymbol{X}^{(\text{joint})}} \log \hat{p}(\boldsymbol{X}^{(\text{joint})})$ is trained, we can obtain the imputation value by simulating the ODE based on Eqs. (9) and (14).

### 4.4 Overall Workflow of NewImp

The computation workflow of NewImp is encapsulated in Algorithm 1. Specifically, we perform a mean imputation to the incomplete matrix $\boldsymbol{X}^{(\text{obs})}$, producing a pre-imputed dataset denoted as $\boldsymbol{X}^{(\text{imp})}$ (step 1). After that, we iteratively conduct DSM training and ODE simulation. In DSM training (steps 3-5), we form $\boldsymbol{X}^{(\text{joint})}$ and conduct DSM on it to acquire a score estimator. In ODE simulation (steps 6-8), we set the starting point and perform ODE simulation, where $u$ is calculated with the score estimator acquired in step 5. The endpoint is treated as the imputation results at the current iteration. After completing $\mathcal{T}$ iterations of this process, the imputed dataset $\hat{\boldsymbol{X}}$ is calculated, where we preserve the observed indices in $\boldsymbol{X}^{(\text{obs})}$ (step 10).

## 5 Experiments

### 5.1 Experimental Setup

**Datasets:** We conduct the case study based on eight real-world datasets from the UCI repository. To simulate missing data, we mask the dataset using a mask matrix, which is realized with a Bernoulli random variable of fixed mean. More detailed information is provided in Appendix E.1.

---

**Algorithm 1** The overall workflow of NewImp approach

---

**Input:** observational data $\boldsymbol{X}^{(\text{obs})}$, mask matrix $\boldsymbol{M}$.
**Hyperparameter:** loop time: $\mathcal{T}$, simulation time: T, discretization step size $\eta$, bandwidth $h$, neural network learning rate $lr$, training epoch $\mathcal{E}$, neural network hidden unit $\text{HU}_{\text{score}}$.
**Output:** imputed data $\hat{\boldsymbol{X}}$.

1: $\boldsymbol{X}^{(\text{imp})} \leftarrow \text{Initialize}(\boldsymbol{X}^{(\text{obs})}) \odot (\mathbb{1}_{\text{N}\times\text{D}} - \boldsymbol{M})$                ▷ Initialization
2: **for** $t = 0$ **to** $\mathcal{T}$ **do**
3:     $\boldsymbol{X}^{(\text{miss})} \leftarrow \boldsymbol{X}^{(\text{imp})}$
4:     $\boldsymbol{X}^{(\text{joint})} \leftarrow \boldsymbol{X}^{(\text{miss})} \odot (\mathbb{1}_{\text{N}\times\text{D}} - \boldsymbol{M}) + \boldsymbol{X}^{(\text{obs})} \odot \boldsymbol{M}$
5:     $\nabla_{\boldsymbol{X}^{(\text{joint})}} \log \hat{p}(\boldsymbol{X}^{(\text{joint})}) \leftarrow \text{DSM}(\boldsymbol{X}^{(\text{joint})})$            ▷ see Algorithm 3
6:     $\boldsymbol{X}_0^{(\text{miss})} \leftarrow \boldsymbol{X}^{(\text{imp})}$
7:     $\boldsymbol{X}_{\text{T}}^{(\text{miss})} \leftarrow \boldsymbol{X}_0^{(\text{miss})} + \int_0^{\text{T}} u(\boldsymbol{X}^{(\text{joint})}) \mathrm{d}\tau$    ▷ ODE Simulation by Appendix D.1 with Eq. (9)
8:     $\boldsymbol{X}^{(\text{imp})} \leftarrow \boldsymbol{X}_{\text{T}}^{(\text{miss})}$
9: **end for**
10: $\hat{\boldsymbol{X}} \leftarrow \boldsymbol{X}^{(\text{obs})} \odot \boldsymbol{M} + \boldsymbol{X}^{(\text{imp})} \odot (\mathbb{1}_{\text{N}\times\text{D}} - \boldsymbol{M})$

---

**Baselines:** We compare NewImp with DM-based MDI models: CSDI for Tabular Data (CSDI_T) [51], MissDiff [41]. In addition, we also select other well-known MDI models like Sinkhorn (Sink) [40], Transform Distribution Matching (TDM) [72], Generative Adversarial Imputation Nets (GAIN) [67], Missing Data Importance-Weighted Autoencoder (MIWAE) [39], Missing data Imputation Refinement And Causal LEarning (MIRACLE) [28], and ReMasker [15]. Details concerning experimental settings are given in Appendix E.2.

**Implementation Details:** In this study, we employ a two-layer multi-layer perceptron to model $\nabla_{\boldsymbol{X}^{(\text{joint})}} \log \hat{p}(\boldsymbol{X}^{(\text{joint})})$. Each layer is configured with 256 hidden units ($\text{HU}_{\text{score}}$), and the activation function is set as the 'Swish' function [43]. For DSM training (step 5 of Algorithm 1), the variance scale $\sigma$ is set as 0.1, the network is trained by the Adam optimizer [26] with the learning rate of $1.0 \times 10^{-3}$, and the batch size is dynamically set to dataset size N. Meanwhile, for the ODE simulation part (step 7 of Algorithm 1), we specify a simulation time (T) of 500, a regularization strength of ($\lambda$) 10.0, a step size of ($\eta$) 0.1, and a bandwidth ($h$) of 0.5. The loop time $\mathcal{T}$ for NewImp is set as 2. Since only missing indices are updated, the evaluation focuses exclusively on imputation errors for these indices. To this end, we modify the mean absolute error (MAE) and squared Wasserstein-2 distance (WASS) according to reference [22] as follows:

$$\text{MAE} := \frac{\sum_{i=1}^{\text{N}} \sum_{j=1}^{\text{D}} [|\boldsymbol{X}_{i,j}^{(\text{ideal})} - \hat{\boldsymbol{X}}_{i,j}| \odot (\mathbb{1}_{\text{N}\times\text{D}} - \boldsymbol{M})_{i,j}]}{\sum_{i=1}^{\text{N}} \sum_{j=1}^{\text{D}} (\mathbb{1}_{\text{N}\times\text{D}} - \boldsymbol{M})_{i,j}},$$

$$\text{WASS} := \mathcal{W}_2^2 \big[\frac{1}{\boldsymbol{m}_1} \sum_{i=1}^{\boldsymbol{m}_1} \varDelta_{[\hat{\boldsymbol{X}}_{\boldsymbol{M}_1}]_{i,:}}, \frac{1}{\boldsymbol{m}_1} \sum_{i=1}^{\boldsymbol{M}_1} \varDelta_{[\boldsymbol{X}_{\boldsymbol{M}_1}^{(\text{ideal})}]_{i,:}} \big],$$

where $\mathcal{W}_2^2$ denotes the squared Wasserstein-2 distance, $\boldsymbol{M}_1 := \{i : \exists j, \boldsymbol{M}_{i,j} = 0\}$ represents the subset of $\boldsymbol{M}_{i,j}$ with at least one missing value, $\boldsymbol{m}_1$ is the number of data points with at least one missing value, and $\varDelta_{\boldsymbol{X}}$ is the Dirac distribution (measure) concentrated on $\boldsymbol{X}$.

### 5.2 Baseline Comparison Results

Table 1 presents the imputation quality of NewImp and other imputation approaches under the MAR and MCAR scenarios. The primary observations are detailed as follows:

- Models with neural architectures such as MIRACLE, MIWAE, and TDM demonstrate superior performance compared to models lacking such architectures. This observation suggests that integrating neural networks into MDI tasks can significantly enhance model performance.

- DM-based imputation approaches generally perform worse than other MDI methods. This outcome indicates that despite the incorporation of complex nonlinear neural architectures to boost performance, employing diversity-oriented generative approaches may not align well with the precision requirements of MDI tasks.

Table 1: Overall performance of MAE and WASS metrics with 30% missing rate.

| Scenario | Model | BT | | BCD | | CC | | CBV | | IS | | PK | | QB | | WQW | |
|---|---|---|---|---|---|---|---|---|---|---|---|---|---|---|---|---|---|
| | | MAE | WASS | MAE | WASS | MAE | WASS | MAE | WASS | MAE | WASS | MAE | WASS | MAE | WASS | MAE | WASS |
| MAR | CSDI_T | 0.93 * | 3.44 * | 0.92 * | 18.20 * | 0.85 * | 2.82 * | 0.81 * | 3.86 * | 0.70 * | 16.86 * | 0.99 * | 15.86 * | 0.65 * | 20.10 * | 0.77 * | 4.13 * |
| | MissDiff | 0.85 * | 2.20 * | 0.91 * | 16.53 * | 0.87 * | 1.59 * | 0.83 * | 3.87 * | 0.72 * | 13.25 * | 0.92 * | 17.07 * | 0.63 * | 26.25 * | 0.75 * | 6.88 * |
| | GAIN | 0.75 * | 0.65 * | 0.54 * | 1.64 * | 0.75 * | 0.67 * | 0.68 * | 0.68 * | 0.56 * | 1.88 * | 0.59 * | 1.90 * | 0.65 * | 5.05 * | 0.68 * | 0.87 * |
| | MIRACLE | 0.62 * | 0.38 | 0.55 * | 1.92 * | 0.43 | 0.25 | 0.55 * | 0.46 * | 3.39 * | 35.06 * | 4.14 * | 34.07 * | 0.46 | 2.87 * | 0.51 * | 0.56 |
| | MIWAE | 0.64 | 0.53 | 0.52 * | 1.54 * | 0.76 * | 0.64 * | 0.82 * | 0.92 * | 0.50 * | 1.87 * | 0.65 * | 1.98 * | 0.55 * | 5.05 * | 0.62 * | 0.75 * |
| | Sink | 0.87 * | 0.92 * | 0.92 * | 3.84 * | 0.88 * | 0.83 * | 0.84 * | 0.98 * | 0.75 * | 2.43 * | 0.94 * | 3.61 * | 0.65 * | 4.71 * | 0.76 * | 1.04 * |
| | TDM | 0.83 * | 0.89 * | 0.83 * | 3.47 * | 0.81 * | 0.73 * | 0.76 * | 0.85 * | 0.62 * | 1.96 * | 0.86 * | 3.36 * | 0.59 * | 4.46 * | 0.73 * | 0.99 * |
| | ReMasker | **0.52** | 0.52 | 0.48 * | 1.15 | 0.60 * | 0.43 * | 0.49 * | 0.37 * | 0.62 * | 2.23 * | 0.61 * | 1.59 * | 0.60 * | 3.81 | 0.51 * | 0.59 * |
| | **NewImp** | 0.52 | **0.38** | **0.34** | **0.82** | **0.35** | 0.25 | **0.31** | **0.20** | **0.39** | **1.31** | **0.44** | **1.21** | **0.45** | 3.50 | **0.46** | **0.55** |
| MCAR | CSDI_T | 0.73 * | 1.93 * | 0.73 * | 15.51 * | 0.85 * | 2.71 * | 0.83 * | 3.79 * | 0.76 * | 15.19 * | 0.72 * | 12.42 * | 0.57 * | 19.89 * | 0.78 * | 4.11 * |
| | MissDiff | 0.72 * | 1.62 * | 0.73 * | 14.39 * | 0.84 * | 1.23 * | 0.82 * | 3.31 * | 0.75 * | 13.01 * | 0.71 * | 14.12 * | 0.56 * | 19.67 * | 0.76 * | 4.95 * |
| | GAIN | 0.72 * | 0.39 * | 0.38 * | 1.41 * | 0.78 * | 0.73 * | 0.72 * | 0.99 * | 0.57 * | 3.72 * | 0.46 * | 1.70 | 0.42 * | 3.62 | 0.73 * | 1.14 * |
| | MIRACLE | 0.52 | 0.15 * | 0.44 * | 1.94 * | 0.53 * | 0.35 | 0.61 * | 0.72 * | 2.99 * | 52.92 * | 3.38 * | 42.78 * | 0.35 | 2.71 * | 0.56 * | **0.75** |
| | MIWAE | 0.58 * | 0.24 | 0.50 * | 2.55 * | 0.76 * | 0.69 * | 0.83 * | 1.24 * | 0.64 * | 4.95 * | 0.51 * | 2.05 * | 0.48 * | 5.87 * | 0.67 * | 0.95 * |
| | Sink | 0.73 * | 0.48 * | 0.75 * | 4.39 * | 0.84 * | 0.85 * | 0.82 * | 1.27 * | 0.75 * | 4.94 * | 0.74 * | 3.36 * | 0.61 * | 5.92 * | 0.76 * | 1.25 * |
| | TDM | 0.68 * | 0.42 * | 0.63 * | 3.57 * | 0.77 * | 0.75 * | 0.77 * | 1.15 * | 0.66 * | 4.20 * | 0.64 * | 2.89 * | 0.52 * | 5.34 * | 0.74 * | 1.20 * |
| | ReMasker | **0.46** * | **0.11** | 0.39 * | 1.69 * | 0.55 * | 0.37 | 0.56 * | 0.64 * | 0.54 * | 4.01 * | 0.48 * | 1.71 * | 0.45 * | 3.94 | 0.57 * | 0.76 |
| | **NewImp** | 0.48 | 0.18 | **0.25** | **0.80** | **0.47** | **0.34** | **0.42** | **0.44** | **0.44** | **3.05** | **0.32** | **1.01** | **0.34** | **3.66** | **0.53** | 0.76 |

*Kindly Note*: The best results are **bolded** and the second best results are underliend. "*" marks the results that NewImp significantly outperform with $p$-value$< 0.05$ over paired samples $t$-test.

Table 2: Ablation results with 30% missing rate.

| Scenario | NER | Joint | BT | | BCD | | CC | | CBV | | IS | | PK | | QB | | WQW | |
|---|---|---|---|---|---|---|---|---|---|---|---|---|---|---|---|---|---|---|---|
| | | | MAE | WASS | MAE | WASS | MAE | WASS | MAE | WASS | MAE | WASS | MAE | WASS | MAE | WASS | MAE | WASS |
| MAR | ✗ | ✗ | 0.96* | 3.82* | 1.05* | 20.2* | 1.04* | 5.47* | 0.86* | 5.81* | 0.67* | 20.2* | 1.06* | 15.6* | 0.72* | 22.5* | 0.79* | 6.49* |
| | ✗ | ✓ | 0.54 | 0.42 | 0.34 | 0.82 | 0.61* | 0.40* | 0.58* | 0.47* | 0.43* | 1.34 | 0.46* | 1.25* | 0.47* | 3.56* | 0.55* | 0.64* |
| | ✓ | ✗ | 0.96* | 3.83* | 1.05* | 20.3* | 1.04* | 5.49* | 0.86* | 5.83* | 0.67* | 20.2* | 1.06* | 15.6* | 0.72* | 22.5* | 0.79* | 6.51* |
| | ✓ | ✓ | **0.52** | **0.38** | **0.34** | **0.82** | **0.35** | **0.25** | **0.31** | **0.20** | **0.39** | **1.31** | **0.44** | **1.21** | **0.45** | **3.50** | **0.46** | **0.55** |
| MCAR | ✗ | ✗ | 0.72* | 2.11* | 0.74* | 16.7* | 0.85* | 3.72* | 0.83* | 5.22* | 0.74* | 18.4* | 0.71* | 12.7* | 0.58* | 20.1* | 0.76* | 5.57* |
| | ✗ | ✓ | 0.52* | 0.17* | **0.25** | **0.79** | 0.62* | 0.46* | 0.61* | 0.71* | 0.46 | 3.05 | 0.34 | 1.09 | 0.36* | 3.74* | 0.58* | 0.82* |
| | ✓ | ✗ | 0.72* | 2.12* | 0.73* | 16.8* | 0.86* | 3.73* | 0.83* | 5.24* | 0.74* | 18.4* | 0.71* | 12.7* | 0.58* | 20.1* | 0.76* | 5.60* |
| | ✓ | ✓ | **0.48** | **0.18** | **0.25** | 0.80 | **0.47** | **0.34** | **0.42** | **0.44** | **0.44** | **3.05** | **0.32** | **1.01** | **0.34** | **3.66** | **0.53** | **0.76** |

*Kindly Note*: The best results are **bolded** and the second best results are underliend. "*" marks the results that NewImp significantly outperform with $p$-value$< 0.05$ over paired samples $t$-test.

- Our proposed NewImp method consistently ranks as the best or second-best across most comparisons under most of the scenarios and datasets. Notably, NewImp significantly outperforms other DM-based MDI approaches, underscoring the effectiveness of our analytical enhancements and innovations in Sections 3.1, 3.2 and 4.2.

## 5.3 Ablation Study Results

In this section, we conduct the ablation study to assess the contributions of two key components in NewImp: the NER term and the joint modeling strategy (referred to as 'Joint'). The results of this study are detailed in Table 2. Analysis of the data between the second and last rows of Table 2 reveals that, for most cases, in the absence of the NER, the proposal distribution $r(\boldsymbol{X}^{(\text{miss})})$ may become pathological, leading to diminished model performance. Additionally, when comparing results from the first, third, and last rows, it becomes evident that modeling the joint distribution directly, rather than inferring it from the conditional distribution, significantly enhances model performance. This finding underscores the effectiveness of the strategies we have implemented, as discussed in Section 4.2. Overall, the ablation study underscores the critical roles of both the NER term and the joint distribution learning strategy in promoting the performance of NewImp.

## 5.4 Sensitivity Analysis Results

In this section, we analyze the impact of key hyperparameters within the NewImp approach, including the bandwidth $h$ of the RBF kernel function, the hidden units $\text{HU}_{\text{score}}$ in the score network, the weight $\lambda$ of the NER term, and the discretization step size $\eta$ for simulating the ODE defined in Eq. (9). The profound influence of these hyperparameters on learning objectives and overall performance is substantiated by the experimental results presented in Fig. 2. Initially, we explore the effects of varying

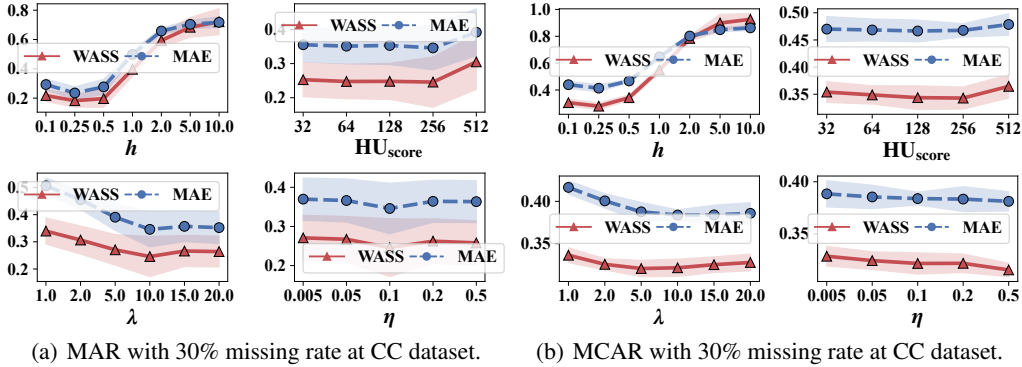

(a) MAR with 30% missing rate at CC dataset.  (b) MCAR with 30% missing rate at CC dataset.

Figure 2: Parameter sensitivity of NewImp on bandwidth for kernel function ($h$), hidden unit of score network $HU_{score}$, NER weight $\lambda$, and discretization step $\eta$ for Eq. (9) on CC dataset. Mean values and one standard deviation from mean are represented by scatters and shaded area, respectively.

the bandwidth $h$. We observe that an increase in bandwidth correlates with a decrease in imputation accuracy. For instance, as the bandwidth increases from 0.5 to 2.0, the MAE and WASS escalate from 0.35 and 0.25 to 0.82 and 0.74, respectively. This trend suggests that excessive bandwidth can lead to an over-smoothed velocity field, expanding the exploration space of the distribution $r(\boldsymbol{X}^{(\text{joint})})$ excessively and failing to adequately 'concentrate' this distribution, ultimately diminishing performance. Subsequently, we examine changes in the score network's hidden units. Increasing the hidden units from 256 to 512 appears to decrease imputation accuracy, likely due to overfitting issues associated with larger neural networks. Next, we adjust the strength of the NER term and find that increasing its intensity generally improves imputation accuracy. This supports the necessity of the NER term, further validating its effectiveness. Lastly, we investigate the discretization step size for the ODE. We find that accuracy initially increases with smaller step sizes but then decreases. This pattern is consistent with ODE simulation behavior, where smaller step sizes require longer to converge, potentially resulting in lower accuracy within a predefined time. Conversely, larger step sizes increase discretization errors, adversely affecting accuracy as well.

## 6 Related Works

### 6.1 Diffusion Models for Missing Data Imputation

The impressive ability of diffusion models to synthesize data [54, 76, 7] has inspired extensive research into their application for MDI tasks [59, 66]. Among the pioneering efforts, the Conditional Score-based Diffusion models for Imputation (CSDI) [51] was the first to adapt diffusion models for time-series MDI, substituting the score function with a conditional distribution and pioneering a novel model training strategy by masking parts of the observational data. Building on this, to address categorical data in tabular datasets, CSDI_T [73] introduced an embedding layer within the feature extractor. To enhance inference efficiency, the conditional Schrödinger bridge method for probabilistic time series imputation proposed modeling the diffusion process as a Schrödinger bridge [10]. Meanwhile, MissDiff [41] utilizes the missing data information as the mask matrix to improve the model training procedure.

Despite these advancements from the perspective of feature extraction module [1, 64], loss function [41], and model inference approach [60], as pointed out by reference [38], the reconciliation of the inherent diversity-seeking nature of diffusion models' generative processes and the accuracy-centric demands of MDI task remains underexplored. To our knowledge, this paper is the first to elucidate the relationship between diffusion models' generative processes and MDI tasks from an optimization perspective (Sections 3.1 and 3.2), which has not been discovered by previous reference [38]. Based on these insights, we further propose our NewImp approach by designing the NER term to prioritize the MDI accuracy (Section 3.2).

## 6.2 Modeling Conditional Distribution by Joint Distribution

Modeling conditional distribution as joint distribution remains an opening question and has a broad potential for application [68, 8, 25]. Conditional sliced WGF [14] first empirically validated that the velocity field of joint distribution and conditional distribution are identical when choosing sliced Wasserstein distance as cost functional. After that, reference [25] extended this relationship and derived the relationship between conditional and joint distribution in various discrepancy metrics like $f$-divergence, Wasserstein distance, and integral probability metrics. On this basis, reference [19] further theoretically proved the equivalence of velocity fields for conditional and joint distribution.

However, the objective of NewImp does not belong to any kind of discrepancy metric [25]. The most similar discrepancy metric is negative KL divergence. Notably, negative KL divergence contains diversity-encouraging 'positive' entropy as the regularization term, and the regularization term in our study is diversity-discouraging 'negative' entropy, and thus more than directly applying these results to our research is needed. On this basis, our theoretical contribution proves that this joint distribution modeling approach can still be applied when the functional is regularized by the negative entropy.

## 6.3 Wasserstein Gradient Flow for Generative Modeling

WGF [2, 46] has been extensively employed in various domains of machine learning, including generative modeling [17, 12, 74, 63], posterior distribution sampling [55, 35, 32, 34], and domain adaptation [75, 36, 37, 71]. In generative modeling [4, 11, 18], the problem is framed as an optimization task, with the objective functional comprising an $f$-divergence term that measures the discrepancy between the proposal distribution and the data distribution, alongside an entropy term that promotes diversity in generative results.

WGF is then utilized to address the optimization of this cost functional, with models being constructed during the solution process. However, as indicated by our illustrative example (Section 3.1), and further supported by our theoretical analysis (Section 3.2), pursuing diversity in accuracy-oriented tasks such as MDI may not be appropriate. Our analysis reveals that the inclusion of an entropy term in the cost functional makes the direct application of diffusion models to MDI tasks unsuitable. Based on these insights, one of our major contributions is demonstrating that WGF can be effectively used to analyze and improve the appropriateness of applying diffusion models to non-generative tasks.

# 7 Conclusions

This work demonstrated that directly applying diffusion models to MDI resulted in suboptimal performance due to unintended diversity and the requirement for data masking, both of which impeded accurate imputation. To counteract this, we proposed NewImp, a novel diffusion model-based MDI approach within the Wasserstein gradient flow framework, designed to suppress unintended diversity. We developed an easy-to-implement form for realizing NewImp in computer code by constraining the velocity field within the reproducing kernel Hilbert space. Furthermore, we proved that the imputation procedure of NewImp could be derived from an equivalent joint-distribution-related functional, thereby obviating the need for data masking. Finally, extensive experiments demonstrated that NewImp effectively mitigates these issues and outperforms prevalent baseline models.

# Acknowledgement

This work was supported in part by the National Natural Science Foundation of China (NSFC) under Grants 62473103 and 623B2002. The first author Zhichao Chen and the corresponding author Hao Wang would like to express their sincere gratitude to PhD Candidate Weiming Liu at Zhejiang University for valuable discussions on the implementation of the FPK equation via ODE/SDE.

*Dedicated to the $100^{th}$ Anniversary of Sun Yat-sen University, the Alma of Zhichao Chen.*

## Footnotes

[2]This paper primarily considers three types of stochastic differential equations (SDEs): variance preserving (VP-SDE), variance exploding (VE-SDE), and sub-VP-SDE, which cover the majority of diffusion models according to Song et al. [50]

[3]$r(\boldsymbol{X}^{(\text{miss})})$ and $u(\boldsymbol{X}^{(\text{miss})})$ are time-varying functions but do not explicitly involve the evolution time $\tau$, thus evolution time $\tau$ is omitted in the input variable.

# References

[1] Juan Lopez Alcaraz and Nils Strodthoff. Diffusion-based time series imputation and forecasting with structured state space models. *Trans. Mach. Learn. Res.*, 2022.

[2] Luigi Ambrosio, Nicola Gigli, and Giuseppe Savaré. *Gradient flows: in metric spaces and in the space of probability measures*. Springer Science & Business Media, 2005.

[3] Brian DO Anderson. Reverse-time diffusion equation models. *Stoch. Process. their Appl.*, 12 (3):313–326, 1982.

[4] Abdul Fatir Ansari, Ming Liang Ang, and Harold Soh. Refining deep generative models via discriminator gradient flow. In *Proc. Int. Conf. Learn. Represent.*, pages 1–24, 2021.

[5] Matthew James Beal. *Variational algorithms for approximate Bayesian inference*. University of London, University College London (United Kingdom), 2003.

[6] John Charles Butcher. *Numerical methods for ordinary differential equations*. John Wiley & Sons, 2016.

[7] Hanqun Cao, Cheng Tan, Zhangyang Gao, Yilun Xu, Guangyong Chen, Pheng-Ann Heng, and Stan Z. Li. A survey on generative diffusion models. *IEEE Trans. Knowl. Data Eng.*, 36(7): 2814–2830, 2024. doi: 10.1109/TKDE.2024.3361474.

[8] Jannis Chemseddine, Paul Hagemann, Christian Wald, and Gabriele Steidl. Conditional wasserstein distances with applications in bayesian ot flow matching. *arXiv preprint arXiv:2403.18705*, pages 1–42, 2024.

[9] Changyou Chen, Ruiyi Zhang, Wenlin Wang, Bai Li, and Liqun Chen. A unified particle-optimization framework for scalable bayesian sampling. *arXiv preprint arXiv:1805.11659*, pages 1–13, 2018.

[10] Yu Chen, Wei Deng, Shikai Fang, Fengpei Li, Nicole Tianjiao Yang, Yikai Zhang, Kashif Rasul, Shandian Zhe, Anderson Schneider, and Yuriy Nevmyvaka. Provably convergent schrödinger bridge with applications to probabilistic time series imputation. In *Proc. Int. Conf. Mach. Learn.*, pages 4485–4513, 2023.

[11] Xiuyuan Cheng, Jianfeng Lu, Yixin Tan, and Yao Xie. Convergence of flow-based generative models via proximal gradient descent in wasserstein space. *IEEE Trans. Inf. Theory*, pages 1–1, 2024. doi: 10.1109/TIT.2024.3422412.

[12] Jaemoo Choi, Jaewoong Choi, and Myungjoo Kang. Scalable Wasserstein gradient flow for generative modeling through unbalanced optimal transport. In *Proc. Int. Conf. Mach. Learn.*, pages 8629–8650, 2024.

[13] Hanze Dong, Xi Wang, LIN Yong, and Tong Zhang. Particle-based variational inference with preconditioned functional gradient flow. In *Proc. Int. Conf. Learn. Represent.*, pages 1–26, 2022.

[14] Chao Du, Tianbo Li, Tianyu Pang, Shuicheng Yan, and Min Lin. Nonparametric generative modeling with conditional sliced-wasserstein flows. In *Proc. Int. Conf. Mach. Learn.*, pages 8565–8584, 2023.

[15] Tianyu Du, Luca Melis Melis, and Ting Wang. Remasker: Imputing tabular data with masked autoencoding. In *Proc. Int. Conf. Learn. Represent.*, pages 1–23, 2024.

[16] Lawrence C Evans. *Partial differential equations*, volume 19. American Mathematical Society, 2022.

[17] Jiaojiao Fan, Qinsheng Zhang, Amirhossein Taghvaei, and Yongxin Chen. Variational Wasserstein gradient flow. In *Proc. Int. Conf. Mach. Learn.*, pages 6185–6215, 2022.

[18] Yuan Gao, Yuling Jiao, Yang Wang, Yao Wang, Can Yang, and Shunkang Zhang. Deep generative learning via variational gradient flow. In *Proc. Int. Conf. Mach. Learn.*, pages 2093–2101, 2019.

[19] Paul Hagemann, Johannes Hertrich, Fabian Altekrüger, Robert Beinert, Jannis Chemseddine, and Gabriele Steidl. Posterior sampling based on gradient flows of the MMD with negative distance kernel. In *Proc. Int. Conf. Learn. Represent.*, pages 1–32, 2024.

[20] Ya-Ping Hsieh, Ali Kavis, Paul Rolland, and Volkan Cevher. Mirrored Langevin Dynamics. In *Proc. Adv. Neural Inf. Process. Syst.*, pages 1–10, 2018.

[21] Aapo Hyvärinen. Estimation of non-normalized statistical models by score matching. *J. Mach. Learn. Res.*, 6(24):695–709, 2005.

[22] Daniel Jarrett, Bogdan C Cebere, Tennison Liu, Alicia Curth, and Mihaela van der Schaar. Hyperimpute: Generalized iterative imputation with automatic model selection. In *Proc. Int. Conf. Mach. Learn.*, pages 9916–9937, 2022.

[23] Richard Jordan, David Kinderlehrer, and Felix Otto. The variational formulation of the fokker–planck equation. *SIAM J. Math. Anal.*, 29(1):1–17, 1998.

[24] Valentin Khrulkov, Gleb Ryzhakov, Andrei Chertkov, and Ivan Oseledets. Understanding DDPM latent codes through optimal transport. In *Proc. Int. Conf. Learn. Represent.*, pages 1–15, 2022.

[25] Young-geun Kim, Kyungbok Lee, and Myunghee Cho Paik. Conditional Wasserstein Generator. *IEEE Trans. Pattern Anal. Mach. Intell.*, 45(6):7208–7219, 2023. doi: 10.1109/TPAMI.2022. 3220965.

[26] Diederik P Kingma and Jimmy Ba. Adam: A method for stochastic optimization. In *Proc. Int. Conf. Learn. Represent.*, pages 1–8, 2015.

[27] Diederik P Kingma and Max Welling. Auto-encoding Variational Bayes. In *Proc. Int. Conf. Learn. Represent.*, pages 1–8, 2014.

[28] Trent Kyono, Yao Zhang, Alexis Bellot, and Mihaela van der Schaar. MIRACLE: Causally-aware imputation via learning missing data mechanisms. *Proc. Adv. Neural Inf. Process. Syst.*, pages 23806–23817, 2021.

[29] Haoxuan Li, Kunhan Wu, Chunyuan Zheng, Yanghao Xiao, Hao Wang, Zhi Geng, Fuli Feng, Xiangnan He, and Peng Wu. Removing hidden confounding in recommendation: a unified multi-task learning approach. *Proc. Adv. Neural Inf. Process. Syst.*, pages 1–13, 2024.

[30] Haoxuan Li, Chunyuan Zheng, Shuyi Wang, Kunhan Wu, Eric Wang, Peng Wu, Zhi Geng, Xu Chen, and Xiao-Hua Zhou. Relaxing the accurate imputation assumption in doubly robust learning for debiased collaborative filtering. In *Proc. Int. Conf. Mach. Learn.*, pages 29448–29460, 2024.

[31] Yingzhen Li and Richard E. Turner. Gradient estimators for implicit models. In *Proc. Int. Conf. Learn. Represent.*, pages 1–19, 2018.

[32] Chang Liu, Jingwei Zhuo, Pengyu Cheng, Ruiyi Zhang, and Jun Zhu. Understanding and accelerating particle-based variational inference. In *Proc. Int. Conf. Mach. Learn.*, pages 4082–4092. PMLR, 2019.

[33] Diju Liu, Yalin Wang, Chenliang Liu, Xiaofeng Yuan, Kai Wang, and Chunhua Yang. Scope-free global multi-condition-aware industrial missing data imputation framework via diffusion transformer. *IEEE Trans. Knowl. Data Eng.*, pages 1–12, 2024. doi: 10.1109/TKDE.2024. 3392897.

[34] Qiang Liu. Stein variational gradient descent as gradient flow. In *Proc. Adv. Neural Inf. Process. Syst.*, volume 30, pages 1–15, 2017.

[35] Qiang Liu and Dilin Wang. Stein variational gradient descent: A general purpose bayesian inference algorithm. In *Proc. Adv. Neural Inf. Process. Syst.*, volume 29, pages 1–13, 2016.

[36] Weiming Liu, Jiajie Su, Chaochao Chen, and Xiaolin Zheng. Leveraging distribution alignment via stein path for cross-domain cold-start recommendation. In *Proc. Adv. Neural Inf. Process. Syst.*, volume 34, pages 19223–19234, 2021.

[37] Weiming Liu, Xiaolin Zheng, Jiajie Su, Longfei Zheng, Chaochao Chen, and Mengling Hu. Contrastive proxy kernel stein path alignment for cross-domain cold-start recommendation. *IEEE Trans. Knowl. Data Eng.*, 35(11):11216–11230, 2023. doi: 10.1109/TKDE.2022.3233789.

[38] Yixin Liu, Thalaiyasingam Ajanthan, Hisham Husain, and Vu Nguyen. Self-supervision improves diffusion models for tabular data imputation. In *Proc. ACM Int. Conf. Inf. Knowl. Manag.*, pages 1–10, 2024.

[39] Pierre-Alexandre Mattei and Jes Frellsen. MIWAE: Deep generative modelling and imputation of incomplete data sets. In *Proc. Int. Conf. Mach. Learn.*, pages 4413–4423, 2019.

[40] Boris Muzellec, Julie Josse, Claire Boyer, and Marco Cuturi. Missing data imputation using optimal transport. In *Proc. Int. Conf. Mach. Learn.*, pages 7130–7140, 2020.

[41] Yidong Ouyang, Liyan Xie, Chongxuan Li, and Guang Cheng. MissDiff: Training diffusion models on tabular data with missing values. In *Proc. Int. Conf. Mach. Learn. Workshop on Structured Probabilistic Inference& Generative Modeling*, 2023.

[42] Adam Paszke, Sam Gross, Francisco Massa, Adam Lerer, James Bradbury, Gregory Chanan, Trevor Killeen, Zeming Lin, Natalia Gimelshein, Luca Antiga, Alban Desmaison, Andreas Kopf, Edward Yang, Zachary DeVito, Martin Raison, Alykhan Tejani, Sasank Chilamkurthy, Benoit Steiner, Lu Fang, Junjie Bai, and Soumith Chintala. PyTorch: An imperative style, high-performance deep learning library. In *Proc. Adv. Neural Inf. Process. Syst.*, volume 32, pages 1–12, 2019.

[43] Prajit Ramachandran, Barret Zoph, and Quoc V Le. Searching for activation functions. *arXiv preprint arXiv:1710.05941*, pages 1–13, 2017.

[44] Donald B Rubin. Inference and missing data. *Biometrika*, 63(3):581–592, 1976.

[45] Walter Rudin et al. *Principles of mathematical analysis*, volume 3. McGraw-hill New York, 1964.

[46] Filippo Santambrogio. {Euclidean, Metric, and Wasserstein} gradient flows: an overview. *Bull. Math. Sci.*, 7:87–154, 2017.

[47] Simo Särkkä and Arno Solin. *Applied stochastic differential equations*, volume 10. Cambridge University Press, 2019.

[48] Jiaxin Shi, Chang Liu, and Lester Mackey. Sampling with Mirrored Stein Operators. *Proc. Int. Conf. Learn. Represent.*, pages 1–26, 2022.

[49] Yang Song, Sahaj Garg, Jiaxin Shi, and Stefano Ermon. Sliced score matching: A scalable approach to density and score estimation. In *Proc. Conf. Uncertainty in Artificial Intelligence*, pages 574–584, 2020.

[50] Yang Song, Jascha Sohl-Dickstein, Diederik P Kingma, Abhishek Kumar, Stefano Ermon, and Ben Poole. Score-based generative modeling through stochastic differential equations. In *Proc. Int. Conf. Learn. Represent.*, pages 1–36, 2020.

[51] Yusuke Tashiro, Jiaming Song, Yang Song, and Stefano Ermon. CSDI: Conditional score-based diffusion models for probabilistic time series imputation. *Proc. Adv. Neural Inf. Process. Syst.*, pages 24804–24816, 2021.

[52] Pascal Vincent. A connection between score matching and denoising autoencoders. *Neural Comput.*, 23(7):1661–1674, 2011.

[53] Dilin Wang and Qiang Liu. Nonlinear stein variational gradient descent for learning diversified mixture models. In *Proc. Int. Conf. Mach. Learn.*, pages 6576–6585, 2019.

[54] Fangyikang Wang, Hubery Yin, Yuejiang Dong, Huminhao Zhu, Chao Zhang, Hanbin Zhao, Hui Qian, and Chen Li. BELM: Bidirectional explicit linear multi-step sampler for exact inversion in diffusion models. In *Proc. Adv. Neural Inf. Process. Syst.*, pages 1–33, 2024.

[55] Fangyikang Wang, Huminhao Zhu, Chao Zhang, Hanbin Zhao, and Hui Qian. GAD-PVI: A general accelerated dynamic-weight particle-based variational inference framework. In *Proc. AAAI Conf. Artif. Intell.*, pages 15466–15473, 2024.

[56] Hao Wang, Jiajun Fan, Zhichao Chen, Haoxuan Li, Weiming Liu, Tianqiao Liu, Quanyu Dai, Yichao Wang, Zhenhua Dong, and Ruiming Tang. Optimal transport for treatment effect estimation. In *Proc. Adv. Neural Inf. Process. Syst.*, pages 5404–5418, 2023.

[57] Hao Wang, Zhichao Chen, Zhaoran Liu, Haozhe Li, Degui Yang, Xinggao Liu, and Haoxuan Li. Entire space counterfactual learning for reliable content recommendations. *IEEE Trans. Inf. Forensics Secur.*, pages 1–1, 2024.

[58] Hao Wang, Zhichao Chen, Zhaoran Liu, Licheng Pan, Hu Xu, Yilin Liao, Haozhe Li, and Xinggao Liu. SPOT-I: Similarity preserved optimal transport for industrial iot data imputation. *IEEE Trans. Ind. Inform.*, pages 1–9, 2024. doi: 10.1109/TII.2024.3452241.

[59] Jun Wang, Wenjie Du, Wei Cao, Keli Zhang, Wenjia Wang, Yuxuan Liang, and Qingsong Wen. Deep learning for multivariate time series imputation: A survey. *arXiv preprint arXiv:2402.04059*, pages 1–9, 2024.

[60] Xu Wang, Hongbo Zhang, Pengkun Wang, Yudong Zhang, Binwu Wang, Zhengyang Zhou, and Yang Wang. An observed value consistent diffusion model for imputing missing values in multivariate time series. In *Proc. ACM SIGKDD Int. Conf. Knowl. Discovery Data Mining*, pages 2409–2418, 2023.

[61] Yifei Wang and Wuchen Li. Accelerated information gradient flow. *J. Sci. Comput.*, 90:1–47, 2022.

[62] Max Welling and Yee W Teh. Bayesian learning via stochastic gradient langevin dynamics. In *Proc. Int. Conf. Mach. Learn.*, pages 681–688. Citeseer, 2011.

[63] Chen Xu, Xiuyuan Cheng, and Yao Xie. Normalizing flow neural networks by JKO scheme. In *Proc. Adv. Neural Inf. Process. Syst.*, pages 47379–47405, 2023.

[64] Jingwen Xu, Fei Lyu, and Pong C Yuen. Density-aware temporal attentive step-wise diffusion model for medical time series imputation. In *Proc. ACM Int. Conf. Inf. Knowl. Manag.*, pages 2836–2845, 2023.

[65] Ling Yang, Zhilong Zhang, Yang Song, Shenda Hong, Runsheng Xu, Yue Zhao, Wentao Zhang, Bin Cui, and Ming-Hsuan Yang. Diffusion models: A comprehensive survey of methods and applications. *ACM Comput. Surv.*, 56(4):1–39, 2023.

[66] Yiyuan Yang, Ming Jin, Haomin Wen, Chaoli Zhang, Yuxuan Liang, Lintao Ma, Yi Wang, Chenghao Liu, Bin Yang, Zenglin Xu, et al. A survey on diffusion models for time series and spatio-temporal data. *arXiv preprint arXiv:2404.18886*, pages 1–27, 2024.

[67] Jinsung Yoon, James Jordon, and Mihaela Schaar. GAIN: Missing data imputation using generative adversarial nets. In *Proc. Int. Conf. Mach. Learn.*, pages 5689–5698. PMLR, 2018.

[68] Shipeng Yu, Kai Yu, Volker Tresp, Hans-Peter Kriegel, and Mingrui Wu. Supervised probabilistic principal component analysis. In *Proc. ACM SIGKDD Int. Conf. Knowl. Discovery Data Mining*, pages 464–473, 2006.

[69] Chao Zhang, Zhijian Li, Xin Du, and Hui Qian. Dpvi: A dynamic-weight particle-based variational inference framework. In *Proc. Int. Joint Conf. Artif. Intell.*, pages 4900–4906, 2022.

[70] Cheng Zhang, Judith Bütepage, Hedvig Kjellström, and Stephan Mandt. Advances in variational inference. *IEEE Trans. Pattern Anal. Mach. Intell.*, 41(8):2008–2026, 2019. doi: 10.1109/TPAMI.2018.2889774.

[71] Yulong Zhang, Shuhao Chen, Weisen Jiang, Yu Zhang, Jiangang Lu, and James T Kwok. Domain-guided conditional diffusion model for unsupervised domain adaptation. *arXiv preprint arXiv:2309.14360*, pages 1–13, 2023.

[72] He Zhao, Ke Sun, Amir Dezfouli, and Edwin V Bonilla. Transformed distribution matching for missing value imputation. In *Proc. Int. Conf. Mach. Learn.*, pages 42159–42186, 2023.

[73] Shuhan Zheng and Nontawat Charoenphakdee. Diffusion models for missing value imputation in tabular data. In *Proc. Adv. Neural Inf. Process. Syst. Workshop on First Table Representation*, 2022.

[74] Huminhao Zhu, Fangyikang Wang, Chao Zhang, Hanbin Zhao, and Hui Qian. Neural sinkhorn gradient flow. *arXiv preprint arXiv:2401.14069*, pages 1–17, 2024.

[75] Zhan Zhuang, Yu Zhang, and Ying Wei. Gradual domain adaptation via gradient flow. In *Proc. Int. Conf. Learn. Represent.*, pages 1–27, 2024.

[76] Zhan Zhuang, Yulong Zhang, Xuehao Wang, Jiangang Lu, Ying Wei, and Yu Zhang. Time-Varying LoRA: Towards effective cross-domain fine-tuning of diffusion models. In *Proc. Adv. Neural Inf. Process. Syst.*, pages 1–25, 2024.

# Appendix Contents

# Appendix A  Detailed Preliminaries of Wasserstein Gradient Flow

In this section, we want to introduce the WGF framework and its application scenarios to better understand this paper. Before introduction, the following concepts are listed to better understand the WGF framework:

1. **Wasserstein Metric:** Let $\mathcal{P}_2(\mathbb{R}^D)$ represent the space of probability measures on $\mathbb{R}^D$ that possess finite second moments. Formally, this is expressed as $\mathcal{P}_2(\mathbb{R}^D) = \{\mu \in \mathcal{M}(\mathbb{R}^D) \mid \int \|x\|^2 \mathrm{d}\mu(x) < \infty\}$, where $\mathcal{M}(\mathbb{R}^D)$ denotes the set of all probability measures on $\mathbb{R}^D$. Considering any two probability measures $\mu, \nu \in \mathcal{P}_2(\mathbb{R}^D)$, we define the Wasserstein-$p$ distance between $\mu$ and $\nu$ as follows:

$$\mathcal{W}_p = \left\{ \inf_{\pi \in \Gamma(\mu, \nu)} \int_{\mathbb{R}^D \times \mathbb{R}^D} \|x - y\|^p \mathrm{d}\pi(x, y) \right\}^{\frac{1}{p}}. \tag{A.1}$$

Here, $\Gamma(\mu, \nu)$ represents the collection of all joint distributions (couplings) between $\mu$ and $\nu$. For every joint distribution $\pi \in \Gamma(\mu, \nu)$, it holds that $\mu(x) = \int_{\mathbb{R}^D} \pi(x, y) \, \mathrm{d}y$ and $\nu(y) = \int_{\mathbb{R}^D} \pi(x, y) \, \mathrm{d}x$. The integral on the right-hand side encapsulates the transportation cost in the optimal transport (OT) problem, framed by Kantorovich's formulation, where $\pi^*$ denotes the optimal transportation plan.

Furthermore, leveraging Jensen's inequality facilitates demonstrating the monotonicity of the Wasserstein-$p$ distance, affirming that for $1 \leq p \leq q$, the relationship $\mathcal{W}_p(\mu, \nu) \leq \mathcal{W}_q(\mu, \nu)$ invariably holds. Building on this principle, we can articulate the inner product within the measurable space $(\mathcal{P}_2(\mathbb{R}^D), \mathcal{W})$ as delineated below:

$$\langle \mu, \nu \rangle_{\mu_\tau} = \int_{\mathbb{R}^D} \langle \mu, \nu \rangle_{\mathbb{R}^D} \mathrm{d}\mu_\tau \tag{A.2}$$

2. **Gradient Flow in Wasserstein Space:** Consider a functional $\mathcal{F}$ associated with $\mu \in \mathcal{P}_2(\mathbb{R}^D)$. Our objective is to identify the optimal $\mu$ that minimizes $\mathcal{F}$:

$$\min_{\mu \in \mathcal{P}_2(\mathbb{R}^D)} \mathcal{F}(\mu) + \text{const.} \tag{A.3}$$

To facilitate the decrease of $\mathcal{F}(\mu)$, we introduce a velocity field $u_\mu : \mathbb{R}^D \to \mathbb{R}^D$ designed to expedite the reduction of $\mathcal{F}(\mu)$ as $\mu$ evolves under this field. Utilizing the chain rule yields:

$$\frac{\mathrm{d}\mathcal{F}(\mu)}{\mathrm{d}\tau} = \int \left\langle \nabla \frac{\delta \mathcal{F}(\mu)}{\delta \mu}, u_\mu \right\rangle \mathrm{d}\mu, \tag{A.4}$$

where $\delta$ represents the first variation operator. To ensure the decrease of $\mathcal{F}(\mu)$, i.e., $\frac{\mathrm{d}\mathcal{F}(\mu)}{\mathrm{d}\tau} \leq 0$, the velocity field is defined as:

$$u_\mu = -\nabla \frac{\delta \mathcal{F}(\mu)}{\delta \mu}. \tag{A.5}$$

The decline of $\mathcal{F}(\mu)$ aligns with the following partial differential equation (PDE) called the continuity equation:

$$\frac{\partial \mu}{\partial \tau} = -\nabla \cdot (\mu u_\mu). \tag{A.6}$$

Hence, the continuity equation Eq. (A.6), coupled with the velocity field articulated in Eq. (A.5), is recognized as the *Wasserstein Gradient Flow*, delineating the steepest descent direction of cost functional $\mathcal{F}(\mu)$ in the Wasserstein space.

3. **Simulation of WGF & Sampling:** There are primarily two discretization techniques for the WGF: the forward scheme and the backward scheme.

   • **Forward Scheme:** The forward scheme applies gradient descent within the Wasserstein space to identify the direction of the steepest descent. For an energy functional $\mathcal{F}(\mu)$ with a specified step size $\eta$, the update rule in the forward scheme is formulated as:

   $$\mu_{\tau+1} = (\mathrm{Id} - \nabla \frac{\delta \mathcal{F}(\mu)}{\delta \mu})_{\#} \mu_\tau, \tag{A.7}$$

   facilitating an intuitive and direct update mechanism that emulates the gradient flow in the Euclidean space but transposed into the Wasserstein space.

- **Backward Scheme:** Conversely, the backward scheme, often referred to as the Jordan-Kinderlehrer-Otto scheme [23], represents a more implicit discretization approach. It defines the subsequent distribution $\mu_{\tau+1}$ by solving an optimization problem that balances the energy decrease and the transportation cost. This scheme is mathematically denoted as:

$$\mu_{\tau+1} = \underset{\mu \in \mathcal{P}_2(\mathbb{R}^D)}{\arg\min} \ \mathcal{F}(\mu) + \frac{1}{2\eta}\mathcal{W}_2^2(\mu, \mu_\tau), \tag{A.8}$$

  thereby integrating the energy minimization and transport efficiency into a single variational problem that reflects the inherent structure of the Wasserstein space.

These schemes provide distinct yet complementary approaches to discretizing the dynamics defined by WGFs, offering different perspectives and tools for the analysis and computation of these flows.

Leveraging the WGF framework, if we designate the functional $\mathcal{F}$ to be the KL divergence, it yields a particular formulation for the velocity field.

$$u_\mu = -\nabla \frac{\delta \mathbb{D}_{\text{KL}}(\mu\|p)}{\delta\mu} = \nabla \log p - \nabla \log \mu. \tag{A.9}$$

On this basis, plug Eq. (A.9) into Eq. (A.6), we can get the following PDE:

$$\frac{\partial\mu}{\partial\tau} = -\nabla \cdot (\mu\nabla \log p) + \nabla \cdot \nabla\mu. \tag{A.10}$$

According to Theorem 5.4 in reference [47], denote the random sample from distribution $p$ as $x$, we can obtain the following stochastic differential equation (SDE) called Langevin equation [62] for implementing this gradient flow easily:

$$\mathrm{d}x = \nabla \log p(x)\mathrm{d}\tau + \sqrt{2}\mathrm{d}W_\tau, \tag{A.11}$$

where $\mathrm{d}W_\tau$ is the standard Wiener process (also known as Brownian motion).

## Appendix B  Detailed Information for Toy Cases in Section 3.1

To investigate what would happen if we directly applied diffusion models to MDI tasks, we consider the following optimization problem:

$$\underset{\boldsymbol{a}_h \in \Delta^2}{\arg\max} \underbrace{\sum_{h=1}^{\text{H}} \left\{ \log \frac{\Gamma\left(\sum_{k=1}^3 \rho_k\right)}{\prod_{k=1}^3 \Gamma(\rho_k)} + \sum_{k=1}^3 (\rho_k - 1) \log \boldsymbol{a}_{k,h} \right\}}_{:=\mathcal{F}_{\text{Dir}}}, \text{H} = 8, \rho_k|_{k=1}^3 = [2.5, 2.5, 5.0],$$

$$\tag{B.1}$$

which corresponds to the density function of a Dirichlet distribution, $\text{Dir}([2.5, 2.5, 5.0])$, where $\boldsymbol{a}_h$ lies on the three-dimensional standard simplex $\Delta^2$. The optimal value of $\mathcal{F}_{\text{Dir}}$ is given by:

$$\left[ \frac{\rho_1 - 1}{\sum_{k=1}^3 \rho_k - 1}, \frac{\rho_2 - 1}{\sum_{k=1}^3 \rho_k - 1}, \frac{\rho_3 - 1}{\sum_{k=1}^3 \rho_k - 1} \right] \approx [0.214, 0.214, 0.571]. \tag{B.2}$$

To optimize this cost functional, we employ the Langevin equation as presented in Eq. (A.11):

$$\mathrm{d}\boldsymbol{a}_h = \nabla_{\boldsymbol{a}_h}\mathcal{F}_{\text{Dir}}\mathrm{d}\tau + \sqrt{2}\mathrm{d}W_\tau, \tag{B.3}$$

and compare the results to Eq. (B.2) to evaluate effectiveness. Additionally, since the support is on a three-dimensional standard simplex $\Delta^2$, to ensure the well-definedness of our approach, we use the mirror descent technique [48, 20], where the Bregman function is defined as $\psi(\boldsymbol{a}_h) = \sum_{k=1}^3 \boldsymbol{a}_{h,k} \log \boldsymbol{a}_{h,k} - \boldsymbol{a}_{h,k}$. Moreover, the optimization of $\boldsymbol{a}_{h,k}|_{h=1}^{\text{H}}|_{k=1}^3$ is conducted by simulating the Langevin equation, which is discretized by the Euler-Maruyama method as follows:

$$\begin{cases} \hat{\boldsymbol{a}}_h^{\tau+1} &= \boldsymbol{a}_h^\tau \times \exp\left(\eta\nabla_{\boldsymbol{a}_h}\mathcal{F}_{\text{Dir}}|_{\boldsymbol{a}_h=\boldsymbol{a}_h^\tau} \times \eta\sqrt{2}\epsilon\right), \epsilon \sim \mathcal{N}(0, I_{3\times3}) \\ \boldsymbol{a}_{h,k}^{\tau+1} &= \dfrac{\hat{\boldsymbol{a}}_{h,k}^{\tau+1}}{\sum_{k=1}^3 \hat{\boldsymbol{a}}_{h,k}^{\tau+1}} \end{cases}. \tag{B.4}$$

With the step size $\eta$ set to $5.0 \times 10^{-3}$, we repeatedly execute Eq. (B.4) 100 times, culminating in the results depicted in Fig. 1 (b).

## Appendix C   Theoretical Analysis

### C.1   Implementation Difficulty of Velocity Field

To the best of our knowledge, the difficulty of implementing the velocity field can be given from two perspectives, namely ODE-based implementation and SDE-based implementation. In this section, we want to discuss these two implementation approaches in detail.

**ODE-based Implementation:**

1. **WGF framework:** According to the continuity equation, we can obtain the following velocity field:

$$\frac{\mathrm{d}\boldsymbol{X}^{(\mathrm{miss})}}{\mathrm{d}\tau} \overset{(i)}{=} u(\boldsymbol{X}^{(\mathrm{miss})}) \overset{(ii)}{=} -[\nabla_{\boldsymbol{X}^{(\mathrm{miss})}} \log \hat{p}(\boldsymbol{X}^{(\mathrm{miss})}|\boldsymbol{X}^{(\mathrm{obs})}) + \lambda\nabla_{\boldsymbol{X}^{(\mathrm{miss})}} \log r(\boldsymbol{X}^{(\mathrm{miss})})], \quad \text{(C.1)}$$

where (i) is based on Section 2.3, and (ii) is based on Section 4.1. The expression of the velocity field involves the computation of density term $r(\boldsymbol{X}^{(\mathrm{miss})})$ [31, 9], which is intractable during practice as we stated in Section 2.3. Based on this, we conclude that implementing this velocity field within the WGF framework is difficult.

2. **Probability flow ODE:** According to reference [50], if we directly plug Eq. (8) into the FPK equation, we can get the following PDE:

$$\begin{aligned}
&\frac{\partial r(\boldsymbol{X}^{(\mathrm{miss})})}{\partial\tau} \\
&= -\nabla \cdot (u(\boldsymbol{X}^{(\mathrm{miss})})r(\boldsymbol{X}^{(\mathrm{miss})})) \\
&= -\left[\nabla_{\boldsymbol{X}^{(\mathrm{miss})}} \log \hat{p}(\boldsymbol{X}^{(\mathrm{miss})}|\boldsymbol{X}^{(\mathrm{obs})})r(\boldsymbol{X}^{(\mathrm{miss})})\right] - \lambda\nabla \cdot \nabla r(\boldsymbol{X}^{(\mathrm{miss})}) \\
&\quad -[\nabla_{\boldsymbol{X}^{(\mathrm{miss})}} \log \hat{p}(\boldsymbol{X}^{(\mathrm{miss})}|\boldsymbol{X}^{(\mathrm{obs})})r(\boldsymbol{X}^{(\mathrm{miss})})] - \lambda\nabla \cdot \nabla r(\boldsymbol{X}^{(\mathrm{miss})}) \\
&= \qquad\qquad\qquad\qquad\qquad\quad \underbrace{+\frac{1}{2}\sigma_\tau^2\nabla \cdot \nabla r(\boldsymbol{X}^{(\mathrm{miss})}) - \frac{1}{2}\sigma_\tau^2\nabla \cdot \nabla r(\boldsymbol{X}^{(\mathrm{miss})})}_{=0} \\
&= -\left\{\left[\nabla_{\boldsymbol{X}^{(\mathrm{miss})}} \log \hat{p}(\boldsymbol{X}^{(\mathrm{miss})}|\boldsymbol{X}^{(\mathrm{obs})}) + (\lambda + \frac{1}{2}\sigma_\tau^2)\nabla \log r(\boldsymbol{X}^{(\mathrm{miss})})\right] r(\boldsymbol{X}^{(\mathrm{miss})})\right\} \\
&\quad + \frac{1}{2}\sigma_\tau^2\nabla \cdot \nabla r(\boldsymbol{X}^{(\mathrm{miss})}).
\end{aligned} \qquad \text{(C.2)}$$

When we set $\sigma_\tau$ as 0, we can find that the corresponding ODE is Eq. (C.1), where we are obliged to compute the intractable density $r(\boldsymbol{X}^{(\mathrm{miss})})$.

**SDE-based Implementation:**
If we plug Section 4.1 into the FPK equation, the corresponding PDE can be given as follows:

$$\begin{aligned}
&\frac{\partial r(\boldsymbol{X}^{(\mathrm{miss})})}{\partial\tau} \\
&= -\nabla \cdot (u(\boldsymbol{X}^{(\mathrm{miss})})r(\boldsymbol{X}^{(\mathrm{miss})})) \\
&= -\left[\nabla_{\boldsymbol{X}^{(\mathrm{miss})}} \log \hat{p}(\boldsymbol{X}^{(\mathrm{miss})}|\boldsymbol{X}^{(\mathrm{obs})})r(\boldsymbol{X}^{(\mathrm{miss})})\right] - \lambda\nabla \cdot \nabla r(\boldsymbol{X}^{(\mathrm{miss})}),
\end{aligned} \qquad \text{(C.3)}$$

where the coefficient before the Laplacian operator $\nabla \cdot \nabla$ is $-1$. To the best of our knowledge, this structure makes deriving a corresponding SDE impossible by current approaches.

### C.2   Proof & Discussions of Concerning Propositions and Corollaries

**Proposition** (3.1). *Within WGF framework, DM-based MDI approaches can be viewed as finding the imputed values $\boldsymbol{X}^{(imp)}$ that maximize the following objective:*

$$\underset{r(\boldsymbol{X}^{(miss)})}{\arg\max} \quad \mathbb{E}_{r(\boldsymbol{X}^{(miss)})}[\log \hat{p}(\boldsymbol{X}^{(miss)}|\boldsymbol{X}^{(obs)})] + \psi(\boldsymbol{X}^{(miss)}) + const, \qquad \text{(C.4)}$$

*where 'const' is the abbreviation of constant, and $\psi(\boldsymbol{X}^{(miss)})$ is a scalar function determined by the type of SDE underlying the DMs.*

- **VP-SDE:** $\psi(\boldsymbol{X}^{(miss)}) = \frac{1}{2}\mathbb{H}[r(\boldsymbol{X}^{(miss)})] + \mathbb{E}_{r(\boldsymbol{X}^{(miss)})}\{\frac{1}{4}[\boldsymbol{X}^{(miss)}]^{\top}[\boldsymbol{X}^{(miss)}]\} \geq 0$

- **VE-SDE:** $\psi(\boldsymbol{X}^{(miss)}) = \frac{1}{2}\mathbb{H}[r(\boldsymbol{X}^{(miss)})] \geq 0$

- **sub-VP-SDE:** $\psi(\boldsymbol{X}^{(miss)}) = \frac{1}{2}\mathbb{H}[r(\boldsymbol{X}^{(miss)})] + \mathbb{E}_{r(\boldsymbol{X}^{(miss)})}\{\frac{1}{4\gamma_{\tau}}[\boldsymbol{X}^{(miss)}]^{\top}[\boldsymbol{X}^{(miss)}]\} \geq 0,$

*where $\mathbb{H}[r(\boldsymbol{X}^{(miss)})] := -\int r(\boldsymbol{X}^{(miss)}) \log r(\boldsymbol{X}^{(miss)}) \mathrm{d}\boldsymbol{X}^{(miss)}$ is the entropy term, $\gamma_{\tau}$ is determined by noise scale $\beta_{\tau}$: $\gamma_{\tau} := (1 - \exp(-2\int_0^{\tau} \beta_s \mathrm{d}s)) > 0, 0 < \beta_1 < \cdots < \beta_T < 1$.*

*Proof.* Since there are various approaches for reversing the sampling procedure of DMs, for simplicity, as we emphasized in Section 3.2, we mainly consider the VP-SDE, VE-SDE, and sub-VP-SDE as analysis objects in this paper.

- **VP-SDE:** According to reference [50], the density evolution of the generative process for VP-SDE can be delineated by the following PDE:

$$
\begin{aligned}
\frac{\partial r(\boldsymbol{X}^{(miss)})}{\partial \tau} = &- \nabla_{\boldsymbol{X}^{(miss)}} \cdot \left\{ r(\boldsymbol{X}^{(miss)})\,[\beta_{\tau}] \left[ \frac{1}{2}\boldsymbol{X}^{(miss)} + \nabla_{\boldsymbol{X}^{(miss)}} \log \hat{p}(\boldsymbol{X}^{(miss)}|\boldsymbol{X}^{(obs)}) \right] \right\} \\
&+ \frac{\beta_{\tau}}{2} \nabla_{\boldsymbol{X}^{(miss)}} \cdot \nabla_{\boldsymbol{X}^{(miss)}} r(\boldsymbol{X}^{(miss)})
\end{aligned}
\tag{C.5}
$$

where $\beta_{\tau} \in (0,1)$ is the time-varying noise scale. On this basis, according to [24], by changing the variable as $\mathrm{d}\tau := \frac{\beta_{\tau}}{2}\mathrm{d}\tau$, we can get the following equation:

$$
\frac{\partial r(\boldsymbol{X}^{(miss)})}{\partial \tau} = -\nabla_{\boldsymbol{X}^{(miss)}} \cdot \left\{ \begin{aligned} & r(\boldsymbol{X}^{(miss)})[\frac{1}{2}\boldsymbol{X}^{(miss)} + \nabla_{\boldsymbol{X}^{(miss)}} \log \hat{p}(\boldsymbol{X}^{(miss)}|\boldsymbol{X}^{(obs)}) \\ & - \frac{1}{2}\nabla_{\boldsymbol{X}^{(miss)}} \log r(\boldsymbol{X}^{(miss)})] \end{aligned} \right\}.
\tag{C.6}
$$

Comparing Eq. (C.6) with Eqs. (A.5) and (A.6), the cost functional to be minimized of this simulation procedure can be given as follows:

$$
\begin{aligned}
\mathcal{F}_{\text{VP-SDE}} = &- \int r(\boldsymbol{X}^{(miss)}) \left\{ \begin{aligned} & \frac{1}{4}[\boldsymbol{X}^{(miss)}]^{\top}[\boldsymbol{X}^{(miss)}] + \log \hat{p}(\boldsymbol{X}^{(miss)}|\boldsymbol{X}^{(obs)}) \\ & - \frac{1}{2} \log r(\boldsymbol{X}^{(miss)}) + \text{const} \end{aligned} \right\} \mathrm{d}\boldsymbol{X}^{(miss)} \\
= &- \mathbb{E}_{r(\boldsymbol{X}^{(miss)})} \left\{ \begin{aligned} & \frac{1}{4}[\boldsymbol{X}^{(miss)}]^{\top}[\boldsymbol{X}^{(miss)}] + \log \hat{p}(\boldsymbol{X}^{(miss)}|\boldsymbol{X}^{(obs)}) \\ & - \frac{1}{2} \log r(\boldsymbol{X}^{(miss)}) + \text{const} \end{aligned} \right\}.
\end{aligned}
\tag{C.7}
$$

Note that $\frac{1}{4}[\boldsymbol{X}^{(miss)}]^{\top}[\boldsymbol{X}^{(miss)}] \geq 0$ and $-\frac{1}{2}\int r(\boldsymbol{X}^{(miss)}) \log r(\boldsymbol{X}^{(miss)}) \mathrm{d}\boldsymbol{X}^{(miss)} \geq 0$ hold, and thus the proposition for VP-SDE is proved by taking the negative of the abovementioned equation.

- **VE-SDE:** Similarly, based on reference [50], the following PDE can be given to delineate the density evolution of the generative process for VE-SDE:

$$
\begin{aligned}
\frac{\partial r(\boldsymbol{X}^{(miss)})}{\partial \tau} = &-\nabla_{\boldsymbol{X}^{(miss)}} \cdot \left\{ r(\boldsymbol{X}^{(miss)}) \left[ -\frac{\mathrm{d}\sigma_{\tau}^2}{\mathrm{d}\tau} \right] \nabla_{\boldsymbol{X}^{(miss)}} \log \hat{p}(\boldsymbol{X}^{(miss)}|\boldsymbol{X}^{(obs)}) \right\} \\
&+ \frac{1}{2}\frac{\mathrm{d}\sigma_{\tau}^2}{\mathrm{d}\tau} \nabla_{\boldsymbol{X}^{(miss)}} \cdot \nabla_{\boldsymbol{X}^{(miss)}} r(\boldsymbol{X}^{(miss)}),
\end{aligned}
\tag{C.8}
$$

where $\sigma_{\tau}^2$ is a time varying noise scale.

As such, by chaning the variable as $\mathrm{d}\tau := \left[ \frac{\mathrm{d}\sigma_{\tau}^2}{\mathrm{d}\tau} \right] \mathrm{d}\tau$ [24], Eq. (C.8) can be reformulated as follows:

$$
\frac{\partial r(\boldsymbol{X}^{(miss)})}{\partial \tau} = -\nabla_{\boldsymbol{X}^{(miss)}} \cdot \left\{ r(\boldsymbol{X}^{(miss)}) \left[ \nabla_{\boldsymbol{X}^{(miss)}} \log \hat{p}(\boldsymbol{X}^{(miss)}|\boldsymbol{X}^{(obs)}) - \frac{1}{2}\nabla_{\boldsymbol{X}^{(miss)}} \log r(\boldsymbol{X}^{(miss)}) \right] \right\}.
\tag{C.9}
$$

Comparing Eq. (C.9) with Eqs. (A.5) and (A.6), the cost functional to be minimized of this simulation procedure can be given as follows:

$$
\begin{aligned}
\mathcal{F}_{\text{VE-SDE}} &= \int r(\boldsymbol{X}^{(\text{miss})}) \left\{ \frac{1}{2} \log r(\boldsymbol{X}^{(\text{miss})}) - \log \hat{p}(\boldsymbol{X}^{(\text{miss})}|\boldsymbol{X}^{(\text{obs})}) + \text{const} \right\} \mathrm{d}\boldsymbol{X}^{(\text{miss})} \\
&= -\mathbb{E}_{r(\boldsymbol{X}^{(\text{miss})})} \left\{ -\frac{1}{2} \log r(\boldsymbol{X}^{(\text{miss})}) + \log \hat{p}(\boldsymbol{X}^{(\text{miss})}|\boldsymbol{X}^{(\text{obs})}) + \text{const} \right\}.
\end{aligned}
\tag{C.10}
$$

Note that the entropy function $-\frac{1}{2}\int r(\boldsymbol{X}^{(\text{miss})}) \log r(\boldsymbol{X}^{(\text{miss})}) \mathrm{d}\boldsymbol{X}^{(\text{miss})} \geq 0$ holds, and thus the proposition for VE-SDE is proved by taking the negative of the abovementioned equation.

- **sub-VP-SDE:** Based on reference [50], the following PDE can be given to delineate the density evolution of the generative process for sub-VP-SDE:

$$
\begin{aligned}
\frac{\partial r(\boldsymbol{X}^{(\text{miss})})}{\partial \tau} &= -\nabla_{\boldsymbol{X}^{(\text{miss})}} \cdot \left\{ r(\boldsymbol{X}^{(\text{miss})}) [\beta_\tau] \left[ \frac{1}{2}\boldsymbol{X}^{(\text{miss})} + \gamma_\tau \nabla_{\boldsymbol{X}^{(\text{miss})}} \log \hat{p}(\boldsymbol{X}^{(\text{miss})}|\boldsymbol{X}^{(\text{obs})}) \right] \right\} \\
&\quad + \frac{\beta_\tau}{2} \gamma_\tau \nabla_{\boldsymbol{X}^{(\text{miss})}} \cdot \nabla_{\boldsymbol{X}^{(\text{miss})}} r(\boldsymbol{X}^{(\text{miss})}),
\end{aligned}
\tag{C.11}
$$

where $\gamma_\tau := (1 - \exp(-2\int_0^\tau \beta_s \mathrm{d}s)) > 0$. On this basis, by chaning the variable as $\mathrm{d}\tau := \frac{\beta_\tau}{2}\mathrm{d}\tau$, we can get the following equation:

$$
\frac{\partial r(\boldsymbol{X}^{(\text{miss})})}{\partial \tau} = -\nabla_{\boldsymbol{X}^{(\text{miss})}} \cdot \left\{ 
\begin{aligned}
&r(\boldsymbol{X}^{(\text{miss})})[\frac{1}{2}\boldsymbol{X}^{(\text{miss})} + \gamma_\tau \nabla_{\boldsymbol{X}^{(\text{miss})}} \log \hat{p}(\boldsymbol{X}^{(\text{miss})}|\boldsymbol{X}^{(\text{obs})}) \\
&\quad - \frac{\gamma_\tau}{2} \nabla_{\boldsymbol{X}^{(\text{miss})}} \log r(\boldsymbol{X}^{(\text{miss})})]
\end{aligned}
\right\}.
\tag{C.12}
$$

Comparing Eq. (C.12) with Eqs. (A.5) and (A.6), the cost functional to be minimized of this simulation procedure can be given as follows:

$$
\begin{aligned}
\mathcal{F}_{\text{sub-VP-SDE}} &= -\int r(\boldsymbol{X}^{(\text{miss})}) \left\{ 
\begin{aligned}
&\frac{1}{4}[\boldsymbol{X}^{(\text{miss})}]^\top[\boldsymbol{X}^{(\text{miss})}] + \gamma_\tau \log \hat{p}(\boldsymbol{X}^{(\text{miss})}|\boldsymbol{X}^{(\text{obs})}) \\
&\quad - \frac{\gamma_\tau}{2} \log r(\boldsymbol{X}^{(\text{miss})}) + \text{const}
\end{aligned}
\right\} \mathrm{d}\boldsymbol{X}^{(\text{miss})} \\
&= -\mathbb{E}_{r(\boldsymbol{X}^{(\text{miss})})} \left\{ 
\begin{aligned}
&\frac{1}{4}[\boldsymbol{X}^{(\text{miss})}]^\top[\boldsymbol{X}^{(\text{miss})}] + \gamma_\tau \log \hat{p}(\boldsymbol{X}^{(\text{miss})}|\boldsymbol{X}^{(\text{obs})}) \\
&\quad - \frac{\gamma_\tau}{2} \log r(\boldsymbol{X}^{(\text{miss})}) + \text{const}
\end{aligned}
\right\} \\
&= -\mathbb{E}_{r(\boldsymbol{X}^{(\text{miss})})} \left\{ 
\begin{aligned}
&\frac{1}{4\gamma_\tau}[\boldsymbol{X}^{(\text{miss})}]^\top[\boldsymbol{X}^{(\text{miss})}] + \log \hat{p}(\boldsymbol{X}^{(\text{miss})}|\boldsymbol{X}^{(\text{obs})}) \\
&\quad - \frac{1}{2} \log r(\boldsymbol{X}^{(\text{miss})}) + \text{const}
\end{aligned}
\right\}.
\end{aligned}
\tag{C.13}
$$

Note that $\frac{1}{4\gamma_\tau}[\boldsymbol{X}^{(\text{miss})}]^\top[\boldsymbol{X}^{(\text{miss})}] \geq 0$ and $-\frac{1}{2}\int r(\boldsymbol{X}^{(\text{miss})}) \log r(\boldsymbol{X}^{(\text{miss})}) \mathrm{d}\boldsymbol{X}^{(\text{miss})} \geq 0$ hold, and thus the proposition for sub-VP-SDE is proved by taking the negative of the abovementioned equation.

In summary, the regularization term $\psi(\boldsymbol{X}^{(\text{miss})})$ for VP-SDE is $\mathbb{E}_{r(\boldsymbol{X}^{(\text{miss})})}\{\frac{1}{4}[\boldsymbol{X}^{(\text{miss})}]^\top[\boldsymbol{X}^{(\text{miss})}]\} + \frac{1}{2}\mathbb{H}[r(\boldsymbol{X}^{(\text{miss})})]$, for VE-SDE is $\frac{1}{2}\mathbb{H}(r(\boldsymbol{X}^{(\text{miss})}))$, and for sub-VP-SDE is $\mathbb{E}_{r(\boldsymbol{X}^{(\text{miss})})}\{\frac{1}{4\gamma_\tau}[\boldsymbol{X}^{(\text{miss})}]^\top[\boldsymbol{X}^{(\text{miss})}]\} + \frac{1}{2}\mathbb{H}[r(\boldsymbol{X}^{(\text{miss})})]$. $\square$

Before proving Proposition 4.1, we want to introduce the following lemma to delineate the evolution of cost functional $\mathcal{F}_{\text{NER}}$ along time $\tau$:

**Lemma C.1.** *The evolution of $\mathcal{F}_{NER}$ along time $\tau$ can be characterized by the following ODE, assuming that the boundary condition $\lim_{\boldsymbol{X}^{(miss)} \to \infty}[u(\boldsymbol{X}^{(miss)})r(\boldsymbol{X}^{(miss)})] = 0$ is satisfied:*

$$
\frac{\mathrm{d}\mathcal{F}_{NER}}{\mathrm{d}\tau} = \mathbb{E}_{r(\boldsymbol{X}^{(miss)})}[u^\top(\boldsymbol{X}^{(miss)})\nabla_{\boldsymbol{X}^{(miss)}} \log \hat{p}(\boldsymbol{X}^{(miss)}|\boldsymbol{X}^{(obs)}) - \lambda \nabla_{\boldsymbol{X}^{(miss)}} \cdot u(\boldsymbol{X}^{(miss)})].
\tag{C.14}
$$

*This boundary condition is achievable, for instance, when $r(\boldsymbol{X}^{(miss)})$ is bounded, and the limit of the velocity field as the norm of $\boldsymbol{X}^{(miss)}$ approaches infinity is zero ($\lim_{\|\boldsymbol{X}^{(miss)}\| \to \infty} u(\boldsymbol{X}^{(miss)}) = 0$).*

*Proof.* Before proving this lemma, we should recognize that the evolution of $\boldsymbol{X}^{(\mathrm{miss})}$ should promise the probability density function $r(\boldsymbol{X}^{(\mathrm{miss})})$ unchanged. In other words, the following continuity equation should be satisfied during the optimization of $r(\boldsymbol{X}^{(\mathrm{miss})})$:

$$\frac{\partial r(\boldsymbol{X}^{(\mathrm{miss})})}{\partial \tau} = -\nabla_{\boldsymbol{X}^{(\mathrm{miss})}} \cdot [r(\boldsymbol{X}^{(\mathrm{miss})})u(\boldsymbol{X}^{(\mathrm{miss})})]. \tag{C.15}$$

On this basis, the evolution of $\mathcal{F}_{\mathrm{NER}}$ along time $\tau$, $\frac{\mathrm{d}\mathcal{F}_{\mathrm{NER}}}{\mathrm{d}\tau}$, can be given as follows based on the chain rule:

$$
\begin{aligned}
&\frac{\mathrm{d}\mathcal{F}_{\mathrm{NER}}}{\mathrm{d}\tau}\\
&= \int \frac{\partial r(\boldsymbol{X}^{(\mathrm{miss})})}{\partial \tau} \left[\log \hat{p}(\boldsymbol{X}^{(\mathrm{miss})}|\boldsymbol{X}^{(\mathrm{obs})}) + \lambda \log r(\boldsymbol{X}^{(\mathrm{miss})}) + \lambda\right]\mathrm{d}\boldsymbol{X}^{(\mathrm{miss})}\\
&= \int -\{\nabla_{\boldsymbol{X}^{(\mathrm{miss})}} \cdot [r(\boldsymbol{X}^{(\mathrm{miss})})u(\boldsymbol{X}^{(\mathrm{miss})})]\}[\log \hat{p}(\boldsymbol{X}^{(\mathrm{miss})}|\boldsymbol{X}^{(\mathrm{obs})}) + \lambda \log r(\boldsymbol{X}^{(\mathrm{miss})}) + \lambda]\mathrm{d}\boldsymbol{X}^{(\mathrm{miss})}\\
&\overset{\text{(i)}}{=} \int [r(\boldsymbol{X}^{(\mathrm{miss})})u(\boldsymbol{X}^{(\mathrm{miss})})]^\top \nabla_{\boldsymbol{X}^{(\mathrm{miss})}}[\log \hat{p}(\boldsymbol{X}^{(\mathrm{miss})}|\boldsymbol{X}^{(\mathrm{obs})}) + \lambda \log r(\boldsymbol{X}^{(\mathrm{miss})}) + \lambda]\mathrm{d}\boldsymbol{X}^{(\mathrm{miss})}\\
&= \int [r(\boldsymbol{X}^{(\mathrm{miss})})u(\boldsymbol{X}^{(\mathrm{miss})})]^\top \{\nabla_{\boldsymbol{X}^{(\mathrm{miss})}}[\log \hat{p}(\boldsymbol{X}^{(\mathrm{miss})}|\boldsymbol{X}^{(\mathrm{obs})}) + \lambda \log r(\boldsymbol{X}^{(\mathrm{miss})})]\}\mathrm{d}\boldsymbol{X}^{(\mathrm{miss})}\\
&= \int [u(\boldsymbol{X}^{(\mathrm{miss})})]^\top [r(\boldsymbol{X}^{(\mathrm{miss})})\nabla_{\boldsymbol{X}^{(\mathrm{miss})}} \log \hat{p}(\boldsymbol{X}^{(\mathrm{miss})}|\boldsymbol{X}^{(\mathrm{obs})}) + \lambda r(\boldsymbol{X}^{(\mathrm{miss})})\nabla_{\boldsymbol{X}^{(\mathrm{miss})}} \log r(\boldsymbol{X}^{(\mathrm{miss})})]\mathrm{d}\boldsymbol{X}^{(\mathrm{miss})}\\
&= \int [u(\boldsymbol{X}^{(\mathrm{miss})})]^\top [r(\boldsymbol{X}^{(\mathrm{miss})})\nabla_{\boldsymbol{X}^{(\mathrm{miss})}} \log \hat{p}(\boldsymbol{X}^{(\mathrm{miss})}|\boldsymbol{X}^{(\mathrm{obs})}) + \lambda\nabla_{\boldsymbol{X}^{(\mathrm{miss})}} r(\boldsymbol{X}^{(\mathrm{miss})})]\mathrm{d}\boldsymbol{X}^{(\mathrm{miss})}\\
&\overset{\text{(ii)}}{=} \int r(\boldsymbol{X}^{(\mathrm{miss})})[u^\top(\boldsymbol{X}^{(\mathrm{miss})})\nabla_{\boldsymbol{X}^{(\mathrm{miss})}} \log \hat{p}(\boldsymbol{X}^{(\mathrm{miss})}|\boldsymbol{X}^{(\mathrm{obs})}) - \lambda\nabla_{\boldsymbol{X}^{(\mathrm{miss})}} \cdot u(\boldsymbol{X}^{(\mathrm{miss})})]\mathrm{d}\boldsymbol{X}^{(\mathrm{miss})}\\
&= \mathbb{E}_{r(\boldsymbol{X}^{(\mathrm{miss})})}[u^\top(\boldsymbol{X}^{(\mathrm{miss})})\nabla_{\boldsymbol{X}^{(\mathrm{miss})}} \log \hat{p}(\boldsymbol{X}^{(\mathrm{miss})}|\boldsymbol{X}^{(\mathrm{obs})}) - \lambda\nabla_{\boldsymbol{X}^{(\mathrm{miss})}} \cdot u(\boldsymbol{X}^{(\mathrm{miss})})],
\end{aligned}
\tag{C.16}
$$

where (i) and (ii) are based on integration by parts. More specifically, when condition $\lim_{\boldsymbol{X}^{(\mathrm{miss})} \to \infty}[u(\boldsymbol{X}^{(\mathrm{miss})})r(\boldsymbol{X}^{(\mathrm{miss})})] = 0$ is satisfied, for example, $r(\boldsymbol{X}^{(\mathrm{miss})})$ is bounded, and the limit of the velocity field as the norm of $\boldsymbol{X}^{(\mathrm{miss})}$ approaches infinity is zero ($\lim_{\|\boldsymbol{X}^{(\mathrm{miss})}\| \to \infty} u(\boldsymbol{X}^{(\mathrm{miss})}) = 0$), we can get the following result [35, 32]:

$$\int \nabla_{\boldsymbol{X}^{(\mathrm{miss})}} \cdot [r(\boldsymbol{X}^{(\mathrm{miss})})u(\boldsymbol{X}^{(\mathrm{miss})})]\mathrm{d}\boldsymbol{X}^{(\mathrm{miss})} = 0,$$

where the left-hand-side can be further decomposed as follows based on the integration by parts:

$$
\begin{aligned}
\int \nabla_{\boldsymbol{X}^{(\mathrm{miss})}} \cdot [r(\boldsymbol{X}^{(\mathrm{miss})})u(\boldsymbol{X}^{(\mathrm{miss})})]\mathrm{d}\boldsymbol{X}^{(\mathrm{miss})} &= \int u^\top(\boldsymbol{X}^{(\mathrm{miss})})\nabla_{\boldsymbol{X}^{(\mathrm{miss})}} r(\boldsymbol{X}^{(\mathrm{miss})})\mathrm{d}\boldsymbol{X}^{(\mathrm{miss})}\\
&\quad + \int [\nabla_{\boldsymbol{X}^{(\mathrm{miss})}} \cdot u(\boldsymbol{X}^{(\mathrm{miss})})]r(\boldsymbol{X}^{(\mathrm{miss})})\mathrm{d}\boldsymbol{X}^{(\mathrm{miss})}.
\end{aligned}
$$

$\square$

Based on Lemma C.1, we can now start proving Proposition 4.1:

**Proposition** (4.1). *Suppose $u(\boldsymbol{X}^{(miss)})$ is a velocity field regularized by the RKHS norm under the following conditions: 1). The kernel function satisfies: $\lim_{\|\boldsymbol{X}^{(miss)}\| \to \infty} K(\boldsymbol{X}^{(miss)}, \tilde{\boldsymbol{X}}^{(miss)}) = 0$. 2). The density $r(\boldsymbol{X}^{(miss)})$ is bounded. Then, the velocity field that minimizes the cost functional $\mathcal{F}_{NER} = \mathbb{E}_{r(\boldsymbol{X}^{(miss)})}[\log \hat{p}(\boldsymbol{X}^{(miss)}|\boldsymbol{X}^{(obs)})] - \lambda\mathbb{H}[r(\boldsymbol{X}^{(miss)})]$ can be given by:*

$$u(\boldsymbol{X}^{(miss)}) = \mathbb{E}_{r(\tilde{\boldsymbol{X}}^{(miss)})}\left\{ \begin{aligned} &-\lambda\nabla_{\tilde{\boldsymbol{X}}^{(miss)}} K(\boldsymbol{X}^{(miss)}, \tilde{\boldsymbol{X}}^{(miss)})\\ &+ [\nabla_{\tilde{\boldsymbol{X}}^{(miss)}} \log \hat{p}(\tilde{\boldsymbol{X}}^{(miss)}|\boldsymbol{X}^{(obs)})]^\top K(\boldsymbol{X}^{(miss)}, \tilde{\boldsymbol{X}}^{(miss)}) \end{aligned} \right\}. \tag{C.17}$$

*where the expectation term $\mathbb{E}_{r(\tilde{\boldsymbol{X}}^{(miss)})}$ can be efficiently estimated using Monte Carlo approximation.*

*Proof.* When the velocity is regularized by the RKHS norm, we can first reformulate Eq. (8) as follows to find the steepest direction for the sake of improving $\mathcal{F}_{\text{NER}}$:

$$u^*(\boldsymbol{X}^{(\text{miss})}) = \underset{u(\boldsymbol{X}^{(\text{miss})})\in\mathcal{H}}{\arg\max} \begin{aligned}&\{\mathbb{E}_{r(\boldsymbol{X}^{(\text{miss})})}[u^\top(\boldsymbol{X}^{(\text{miss})})\nabla_{\boldsymbol{X}^{(\text{miss})}}\log\hat{p}(\boldsymbol{X}^{(\text{miss})}|\boldsymbol{X}^{(\text{obs})})\\&\quad - \lambda\nabla_{\boldsymbol{X}^{(\text{miss})}}\cdot u(\boldsymbol{X}^{(\text{miss})})]\} - \frac{1}{2}\|u(\boldsymbol{X}^{(\text{miss})})\|_{\mathcal{H}}^2.\end{aligned} \tag{C.18}$$

Based on this, assume we have a map function $\phi(x)$, the kernel function can be given as follows:

$$K(x,y) = \langle\phi(x),\phi(y)\rangle_{\mathcal{H}}. \tag{C.19}$$

Hence, the regularization term that control the magnitude of $u(\boldsymbol{X}^{(\text{miss})})$ can be given by $\frac{1}{2}\|u(\boldsymbol{X}^{(\text{miss})})\|_{\mathcal{H}}^2$, and the spectral decomposition of kernel function can be given as follows:

$$K(x,y) = \sum_{i=1}^{\infty}\xi_i\phi_i(x)\phi_i(y), \tag{C.20}$$

where $\phi_i(\cdot)$ indicates the orthonormal basis and $\xi_i$ is the corresponding eigen-value. For any function $u(\boldsymbol{X}^{(\text{miss})})\in\mathcal{H}$, the following decomposition is given:

$$u(\boldsymbol{X}^{(\text{miss})}) = \sum_{i=1}^{\infty}u_i\sqrt{\xi_i}\phi_i(\boldsymbol{X}^{(\text{miss})}), \tag{C.21}$$

where $u_i$ and $\sum_{i=1}^{\infty}\|u_i\|^2 < \infty$.

The learning objective defined in Eq. (8) can be reformulated as follows:

$$\begin{aligned}&u^*(\boldsymbol{X}^{(\text{miss})})\\&= \underset{u(\boldsymbol{X}^{(\text{miss})})\in\mathcal{H}}{\arg\max}\begin{aligned}&\{\mathbb{E}_{r(\boldsymbol{X}^{(\text{miss})})}[u^\top(\boldsymbol{X}^{(\text{miss})})\nabla_{\boldsymbol{X}^{(\text{miss})}}\log\hat{p}(\boldsymbol{X}^{(\text{miss})}|\boldsymbol{X}^{(\text{obs})})\\&\quad - \lambda\nabla_{\boldsymbol{X}^{(\text{miss})}}\cdot u(\boldsymbol{X}^{(\text{miss})})]\} - \frac{1}{2}\|u(\boldsymbol{X}^{(\text{miss})})\|_{\mathcal{H}}^2,\end{aligned}\\&\overset{\text{(i)}}{=} \underset{u(\boldsymbol{X}^{(\text{miss})})\in\mathcal{H}}{\arg\max}\begin{aligned}&\{\mathbb{E}_{r(\tilde{\boldsymbol{X}}^{(\text{miss})})}[\sum_{i=1}^{\infty}\sqrt{\xi_i}\nabla_{\tilde{\boldsymbol{X}}^{(\text{miss})}}\log\hat{p}(\tilde{\boldsymbol{X}}^{(\text{miss})}|\boldsymbol{X}^{(\text{obs})})^\top u_i\phi_i(\tilde{\boldsymbol{X}}^{(\text{miss})})\\&\quad - \lambda\nabla_{\tilde{\boldsymbol{X}}^{(\text{miss})}}\cdot\sum_{i=1}^{\infty}u_i\sqrt{\xi_i}\phi_i(\tilde{\boldsymbol{X}}^{(\text{miss})})]\} - \frac{1}{2}\sum_{i=1}^{\infty}\|u_i\|^2,\end{aligned}\end{aligned} \tag{C.22}$$

Take the right-hand-side of (i) with-respect-to $u_i$, and set it to 0, we can get:

$$\sqrt{\xi_i}\{\mathbb{E}_{r(\tilde{\boldsymbol{X}}^{(\text{miss})})}[[\nabla_{\tilde{\boldsymbol{X}}^{(\text{miss})}}\log\hat{p}(\tilde{\boldsymbol{X}}^{(\text{miss})}|\boldsymbol{X}^{(\text{obs})})]^\top\phi_i(\tilde{\boldsymbol{X}}^{(\text{miss})}) - \lambda\nabla_{\tilde{\boldsymbol{X}}^{(\text{miss})}}\phi_i(\tilde{\boldsymbol{X}}^{(\text{miss})})]\} - u_i = 0. \tag{C.23}$$

On this basis, $u_i^*$ can be given as follows:

$$u_i^* = \sqrt{\xi_i}\{\mathbb{E}_{r(\tilde{\boldsymbol{X}}^{(\text{miss})})}[[\nabla_{\tilde{\boldsymbol{X}}^{(\text{miss})}}\log\hat{p}(\tilde{\boldsymbol{X}}^{(\text{miss})}|\boldsymbol{X}^{(\text{obs})})]^\top\phi_i(\boldsymbol{X}^{(\text{miss})}) - \lambda\nabla_{\tilde{\boldsymbol{X}}^{(\text{miss})}}\phi_i(\tilde{\boldsymbol{X}}^{(\text{miss})})]\}, \tag{C.24}$$

and hence, $u(\boldsymbol{X}^{(\text{miss})})$ can be given as follows:

$$\begin{aligned}&u^*(\boldsymbol{X}^{(\text{miss})})\\&= \sum_{i=1}^{\infty}\sqrt{\xi_i}u_i^*\phi_i(\boldsymbol{X}^{(\text{miss})})\\&= \mathbb{E}_{r(\tilde{\boldsymbol{X}}^{(\text{miss})})}\left[\begin{array}{l}-\lambda\nabla_{\tilde{\boldsymbol{X}}^{(\text{miss})}}K(\boldsymbol{X}^{(\text{miss})},\tilde{\boldsymbol{X}}^{(\text{miss})})\\+ [\nabla_{\tilde{\boldsymbol{X}}^{(\text{miss})}}\log\hat{p}(\tilde{\boldsymbol{X}}^{(\text{miss})}|\boldsymbol{X}^{(\text{obs})})]^\top K(\boldsymbol{X}^{(\text{miss})},\tilde{\boldsymbol{X}}^{(\text{miss})})\end{array}\right].\end{aligned} \tag{C.25}$$

$\square$

**Proposition** (4.2). *Assume that the proposal distribution $r(\boldsymbol{X}^{(joint)})$ is factorized by $r(\boldsymbol{X}^{(joint)}) := r(\boldsymbol{X}^{(miss)})p(\boldsymbol{X}^{(obs)})$. The cost functional associated with the joint distribution is defined as follows:*

$$\mathcal{F}_{joint\text{-}NER} := \mathbb{E}_{r(\boldsymbol{X}^{(joint)})}[\log\hat{p}(\boldsymbol{X}^{(joint)})] - \lambda\mathbb{H}[r(\boldsymbol{X}^{(joint)})], \tag{C.26}$$

*which leads to the velocity field delineated in Eq. (9) and establishes $\mathcal{F}_{joint\text{-}NER}$ as a lower bound for $\mathcal{F}_{NER}$, with the difference being a constant (i.e., $\mathcal{F}_{joint\text{-}NER} = \mathcal{F}_{NER} - const, const \geq 0$).*

Before proving this proposition, we want to first clarify the justification of the assumption that $r(\boldsymbol{X}^{(\text{joint})}) := r(\boldsymbol{X}^{(\text{miss})})r(\boldsymbol{X}^{(\text{obs})}) = r(\boldsymbol{X}^{(\text{miss})})p(\boldsymbol{X}^{(\text{obs})})$. In this part, we set $r(\boldsymbol{X}^{(\text{obs})}) = p(\boldsymbol{X}^{(\text{obs})})$. Before stating the justification, we should come to the following agreements:

1. Throughout the imputation procedure, $\boldsymbol{X}^{(\text{obs})}$ remains invariant regardless of any modifications to $\boldsymbol{X}^{(\text{miss})}$.

2. Given this invariance, it is justified to state that $r(\boldsymbol{X}^{(\text{obs})})$ remains constant from the perspective of particle variational inference represented by reference [35, 34], and consequently $r(\boldsymbol{X}^{(\text{obs})}|\boldsymbol{X}^{(\text{miss})}) = r(\boldsymbol{X}^{(\text{obs})})$, reflecting the independence of $\boldsymbol{X}^{(\text{obs})}$ from $\boldsymbol{X}^{(\text{miss})}$.

Based on this, we want to show that within the WGF framework, the factorizations $r(\boldsymbol{X}^{(\text{miss})})r(\boldsymbol{X}^{(\text{obs})})$ and $r(\boldsymbol{X}^{(\text{miss})}|\boldsymbol{X}^{(\text{obs})})r(\boldsymbol{X}^{(\text{obs})})$ are equivalent. To this end, let us write down the evolution of $r(\boldsymbol{X}^{(\text{obs})}, \boldsymbol{X}^{(\text{miss})})$ along time $\tau$ as follows when $r(\boldsymbol{X}^{(\text{obs})}, \boldsymbol{X}^{(\text{miss})})$ is factorized by $r(\boldsymbol{X}^{(\text{obs})}, \boldsymbol{X}^{(\text{miss})}) = r(\boldsymbol{X}^{(\text{miss})}|\boldsymbol{X}^{(\text{obs})})r(\boldsymbol{X}^{(\text{obs})})$:

$$
\frac{\partial r(\boldsymbol{X}^{(\text{obs})}, \boldsymbol{X}^{(\text{miss})})}{\partial \tau} = \frac{\partial r(\boldsymbol{X}^{(\text{miss})}|\boldsymbol{X}^{(\text{obs})})r(\boldsymbol{X}^{(\text{obs})})}{\partial \tau}
$$

$$
= \underbrace{r(\boldsymbol{X}^{(\text{obs})})\frac{\partial r(\boldsymbol{X}^{(\text{miss})}|\boldsymbol{X}^{(\text{obs})})}{\partial \tau}}_{r(\boldsymbol{X}^{(\text{miss})}|\boldsymbol{X}^{(\text{obs})}) = \frac{r(\boldsymbol{X}^{(\text{obs})}|\boldsymbol{X}^{(\text{miss})})r(\boldsymbol{X}^{(\text{miss})})}{r(\boldsymbol{X}^{(\text{obs})})}} + \underbrace{r(\boldsymbol{X}^{(\text{miss})}|\boldsymbol{X}^{(\text{obs})})\frac{\partial r(\boldsymbol{X}^{(\text{obs})})}{\partial \tau}}_{0},
$$

where the first underbrace is the Bayesian formula, and the second underbrace is based on the abovementioned Agreement 2. Consequently, we can further obtain the following results:

$$
\frac{\partial r(\boldsymbol{X}^{(\text{obs})}, \boldsymbol{X}^{(\text{miss})})}{\partial \tau} = \underbrace{\frac{r(\boldsymbol{X}^{(\text{obs})})}{r(\boldsymbol{X}^{(\text{obs})})}\frac{\partial r(\boldsymbol{X}^{(\text{obs})}|\boldsymbol{X}^{(\text{miss})})r(\boldsymbol{X}^{(\text{miss})})}{\partial \tau}}_{r(\boldsymbol{X}^{(\text{obs})}|\boldsymbol{X}^{(\text{miss})}) = r(\boldsymbol{X}^{(\text{obs})})}
$$

$$
= r(\boldsymbol{X}^{(\text{obs})})\frac{\partial r(\boldsymbol{X}^{(\text{miss})})}{\partial \tau} \tag{C.27}
$$

Similarly, when we factorize $r(\boldsymbol{X}^{(\text{obs})}, \boldsymbol{X}^{(\text{miss})})$ by $r(\boldsymbol{X}^{(\text{obs})}, \boldsymbol{X}^{(\text{miss})}) = r(\boldsymbol{X}^{(\text{miss})})r(\boldsymbol{X}^{(\text{obs})})$, we can get the following result:

$$
\frac{\partial r(\boldsymbol{X}^{(\text{obs})}, \boldsymbol{X}^{(\text{miss})})}{\partial \tau} = \frac{\partial r(\boldsymbol{X}^{(\text{obs})})r(\boldsymbol{X}^{(\text{miss})})}{\partial \tau}
$$

$$
= r(\boldsymbol{X}^{(\text{obs})})\frac{\partial r(\boldsymbol{X}^{(\text{miss})})}{\partial \tau}. \tag{C.28}
$$

Comparing Eq. (C.28) to Eq. (C.27), we can demonstrate our justification of the factorization $r(\boldsymbol{X}^{(\text{obs})}, \boldsymbol{X}^{(\text{miss})}) = r(\boldsymbol{X}^{(\text{miss})})r(\boldsymbol{X}^{(\text{obs})})$. Finally, we would like to conclude with a metaphor to further illustrate the plausibility of this mean-filed factorization, which has been widely applied in variational inference [5]:

1. Consider $r$ as an actor in a play, capable of being molded and shaped. Initially, the actor may not fully embody the role, akin to $r(\boldsymbol{X}^{(\text{miss})})$ not containing information about $\boldsymbol{X}^{(\text{obs})}$.

2. However just as a director shapes an actor's performance through guidance and rehearsal, all we need to do is ensure that $r(\boldsymbol{X}^{(\text{miss})})$ is appropriately molded by the directorial guidance (mirrors the continuity equation $\frac{\partial r}{\partial \tau} = -\nabla \cdot (ur)$) of the velocity field $u$ and the script provided by the critic $p(\boldsymbol{X}^{(\text{obs})}, \boldsymbol{X}^{(\text{miss})})/p(\boldsymbol{X}^{(\text{obs})}|\boldsymbol{X}^{(\text{miss})})$.

3. As long as $r$ can adapt based on this feedback (akin to the WGF framework), it can overcome the limitations of its initial portrayal (akin to $r(\boldsymbol{X}^{(\text{joint})}) = r(\boldsymbol{X}^{(\text{miss})})r(\boldsymbol{X}^{(\text{obs})})$ ).

Based on the abovementioned analysis, we can now start the proof of Proposition 4.2:

*Proof.* Our proof will be divided into two parts namely 'velocity field derivation' and 'upper bound acquirement'.

**Velocity Field Derivation:**

the following continuity equation should be satisfied during the optimization of $r(\boldsymbol{X}^{(\text{miss})})$:

$$\frac{\partial r(\boldsymbol{X}^{(\text{miss})})}{\partial \tau} = -\nabla_{\boldsymbol{X}^{(\text{miss})}} \cdot [r(\boldsymbol{X}^{(\text{miss})})u(\boldsymbol{X}^{(\text{miss})})]$$

$$\Rightarrow \frac{\partial r(\boldsymbol{X}^{(\text{miss})})}{\partial \tau} \times p(\boldsymbol{X}^{(\text{obs})}) = -\nabla_{\boldsymbol{X}^{(\text{miss})}} \cdot [r(\boldsymbol{X}^{(\text{miss})})u(\boldsymbol{X}^{(\text{miss})})] \times p(\boldsymbol{X}^{(\text{obs})}) \qquad \text{(C.29)}$$

$$\overset{\text{(i)}}{\Rightarrow} \frac{\partial r(\boldsymbol{X}^{(\text{joint})})}{\partial \tau} = -\nabla_{\boldsymbol{X}^{(\text{miss})}} \cdot [r(\boldsymbol{X}^{(\text{joint})})u(\boldsymbol{X}^{(\text{joint})})],$$

where (i) is based on the fact that $\boldsymbol{X}^{(\text{obs})}$ remains unchanged during the imputation process. Thus, according to Eq. (C.16), the evolution of $\mathcal{F}_{\text{joint-NER}}$ along time $\tau$, $\frac{\mathrm{d}\mathcal{F}_{\text{joint-NER}}}{\mathrm{d}\tau}$, can be given as follows based on the chain rule:

$$\frac{\mathrm{d}\mathcal{F}_{\text{joint-NER}}}{\mathrm{d}\tau}$$

$$= \int \frac{\partial r(\boldsymbol{X}^{(\text{joint})})}{\partial \tau} \left[ \log \hat{p}(\boldsymbol{X}^{(\text{joint})}) + \lambda \log r(\boldsymbol{X}^{(\text{joint})}) + \lambda \right] \mathrm{d}\boldsymbol{X}^{(\text{joint})}$$

$$= \int -\{\nabla_{\boldsymbol{X}^{(\text{miss})}} \cdot [r(\boldsymbol{X}^{(\text{joint})})u(\boldsymbol{X}^{(\text{joint})})]\}[\log \hat{p}(\boldsymbol{X}^{(\text{joint})}) + \lambda \log r(\boldsymbol{X}^{(\text{joint})}) + \lambda]\mathrm{d}\boldsymbol{X}^{(\text{joint})}$$

$$\overset{\text{(i)}}{=} \int [r(\boldsymbol{X}^{(\text{joint})})u(\boldsymbol{X}^{(\text{joint})})]^{\top} \nabla_{\boldsymbol{X}^{(\text{miss})}}[\log \hat{p}(\boldsymbol{X}^{(\text{joint})}) + \lambda \log r(\boldsymbol{X}^{(\text{joint})}) + \lambda]\mathrm{d}\boldsymbol{X}^{(\text{joint})}$$

$$= \int [r(\boldsymbol{X}^{(\text{joint})})u(\boldsymbol{X}^{(\text{joint})})]^{\top} \{\nabla_{\boldsymbol{X}^{(\text{miss})}}[\log \hat{p}(\boldsymbol{X}^{(\text{joint})}) + \lambda \log r(\boldsymbol{X}^{(\text{joint})})]\}\mathrm{d}\boldsymbol{X}^{(\text{joint})}$$

$$= \int [u(\boldsymbol{X}^{(\text{joint})})]^{\top} [r(\boldsymbol{X}^{(\text{joint})})\nabla_{\boldsymbol{X}^{(\text{miss})}} \log \hat{p}(\boldsymbol{X}^{(\text{joint})}) + \lambda r(\boldsymbol{X}^{(\text{joint})})\nabla_{\boldsymbol{X}^{(\text{miss})}} \log r(\boldsymbol{X}^{(\text{joint})})]\mathrm{d}\boldsymbol{X}^{(\text{joint})}$$

$$= \int [u(\boldsymbol{X}^{(\text{joint})})]^{\top} [r(\boldsymbol{X}^{(\text{joint})})\nabla_{\boldsymbol{X}^{(\text{miss})}} \log \hat{p}(\boldsymbol{X}^{(\text{joint})}) + \lambda \nabla_{\boldsymbol{X}^{(\text{miss})}} r(\boldsymbol{X}^{(\text{joint})})]\mathrm{d}\boldsymbol{X}^{(\text{joint})}$$

$$\overset{\text{(ii)}}{=} \int r(\boldsymbol{X}^{(\text{joint})})[u^{\top}(\boldsymbol{X}^{(\text{joint})})\nabla_{\boldsymbol{X}^{(\text{miss})}} \log \hat{p}(\boldsymbol{X}^{(\text{joint})}) - \lambda \nabla_{\boldsymbol{X}^{(\text{miss})}} \cdot u(\boldsymbol{X}^{(\text{joint})})]\mathrm{d}\boldsymbol{X}^{(\text{joint})}$$

$$= \mathbb{E}_{r(\boldsymbol{X}^{(\text{joint})})}[u^{\top}(\boldsymbol{X}^{(\text{joint})})\nabla_{\boldsymbol{X}^{(\text{miss})}} \log \hat{p}(\boldsymbol{X}^{(\text{joint})}) - \lambda \nabla_{\boldsymbol{X}^{(\text{miss})}} \cdot u(\boldsymbol{X}^{(\text{joint})})],$$

(C.30)

where (i) and (ii) are based on integration by parts. More specifically, when condition $\lim_{\boldsymbol{X}^{(\text{joint})} \to \infty}[u(\boldsymbol{X}^{(\text{joint})})r(\boldsymbol{X}^{(\text{joint})})] = 0$ is satisfied, for example, $r(\boldsymbol{X}^{(\text{joint})})$ is bounded, and the limit of the velocity field as the norm of $\boldsymbol{X}^{(\text{joint})}$ approaches infinity is zero ($\lim_{\|\boldsymbol{X}^{(\text{joint})}\| \to \infty} u(\boldsymbol{X}^{(\text{joint})}) = 0$), we can get the following result [35, 32]:

$$\int \nabla_{\boldsymbol{X}^{(\text{miss})}} \cdot [r(\boldsymbol{X}^{(\text{joint})})u(\boldsymbol{X}^{(\text{joint})})]\mathrm{d}\boldsymbol{X}^{(\text{joint})} = 0,$$

where we omit the gradient operator with respect to the observed variables $\boldsymbol{X}^{(\text{obs})}$, denoted as $\nabla_{\boldsymbol{X}^{(\text{obs})}}$, because $\boldsymbol{X}^{(\text{obs})}$ remains constant during the imputation process. This constancy implies that the divergence $\nabla_{\boldsymbol{X}^{(\text{obs})}} \cdot [r(\boldsymbol{X}^{(\text{joint})})u(\boldsymbol{X}^{(\text{joint})})] = 0$. Consequently, the left-hand-side of this equation can be further decomposed as follows based on the integration by parts:

$$\int \nabla_{\boldsymbol{X}^{(\text{miss})}} \cdot [r(\boldsymbol{X}^{(\text{joint})})u(\boldsymbol{X}^{(\text{joint})})]\mathrm{d}\boldsymbol{X}^{(\text{joint})} = \int u^{\top}(\boldsymbol{X}^{(\text{joint})})\nabla_{\boldsymbol{X}^{(\text{miss})}} r(\boldsymbol{X}^{(\text{joint})})\mathrm{d}\boldsymbol{X}^{(\text{joint})}$$

$$+ \int [\nabla_{\boldsymbol{X}^{(\text{miss})}} \cdot u(\boldsymbol{X}^{(\text{joint})})]r(\boldsymbol{X}^{(\text{joint})})\mathrm{d}\boldsymbol{X}^{(\text{joint})}.$$

Similar to the proof of proposition 4.1, we can restrict the velocity field in RKHS and find the steepest gradient boosting direction as follows according to Eqs. (C.19) to (C.21):

$$
\begin{aligned}
& u^*(\boldsymbol{X}^{(\text{joint})}) \\
&= \underset{u(\boldsymbol{X}^{(\text{joint})})\in\mathcal{H}^{\mathrm{D}}}{\arg\max} \quad
\begin{aligned}
&\{\mathbb{E}_{r(\boldsymbol{X}^{(\text{joint})})}[u^\top(\boldsymbol{X}^{(\text{joint})})\nabla_{\boldsymbol{X}^{(\text{miss})}}\log\hat{p}(\boldsymbol{X}^{(\text{joint})}) \\
&\quad - \lambda\nabla_{\boldsymbol{X}^{(\text{miss})}}\cdot u(\boldsymbol{X}^{(\text{miss})})]\} - \frac{1}{2}\|u(\boldsymbol{X}^{(\text{joint})})\|^2_{\mathcal{H}^{\mathrm{D}}},
\end{aligned} \\
&\overset{(i)}{=} \underset{u(\boldsymbol{X}^{(\text{joint})})\in\mathcal{H}^{\mathrm{D}}}{\arg\max} \quad
\begin{aligned}
&\{\mathbb{E}_{r(\tilde{\boldsymbol{X}}^{(\text{joint})})}[\sum_{i=1}^{\infty}\sqrt{\xi_i}\nabla_{\tilde{\boldsymbol{X}}^{(\text{miss})}}\log\hat{p}(\tilde{\boldsymbol{X}}^{(\text{joint})})^\top u_i\phi_i(\tilde{\boldsymbol{X}}^{(\text{joint})}) \\
&\quad - \lambda\nabla_{\tilde{\boldsymbol{X}}^{(\text{miss})}}\cdot\sum_{i=1}^{\infty}u_i\sqrt{\xi_i}\phi_i(\tilde{\boldsymbol{X}}^{(\text{joint})})]\} - \frac{1}{2}\sum_{i=1}^{\infty}\|u_i\|^2,
\end{aligned}
\end{aligned} \tag{C.31}
$$

Take the right-hand-side of (i) with-respect-to $u_i$, and set it to 0, we can get:

$$
\sqrt{\xi_i}\{\mathbb{E}_{r(\tilde{\boldsymbol{X}}^{(\text{joint})})}[[\nabla_{\tilde{\boldsymbol{X}}^{(\text{miss})}}\log\hat{p}(\tilde{\boldsymbol{X}}^{(\text{joint})})]^\top\phi_i(\tilde{\boldsymbol{X}}^{(\text{joint})}) - \lambda\nabla_{\tilde{\boldsymbol{X}}^{(\text{miss})}}\phi_i(\tilde{\boldsymbol{X}}^{(\text{joint})})]\} - u_i = 0. \tag{C.32}
$$

On this basis, $u_i^*$ can be given as follows:

$$
u_i^* = \sqrt{\xi_i}\{\mathbb{E}_{r(\tilde{\boldsymbol{X}}^{(\text{joint})})}[[\nabla_{\tilde{\boldsymbol{X}}^{(\text{miss})}}\log\hat{p}(\tilde{\boldsymbol{X}}^{(\text{joint})})]^\top\phi_i(\tilde{\boldsymbol{X}}^{(\text{joint})}) - \lambda\nabla_{\tilde{\boldsymbol{X}}^{(\text{miss})}}\phi_i(\tilde{\boldsymbol{X}}^{(\text{joint})})]\}, \tag{C.33}
$$

and hence, $u(\boldsymbol{X}^{(\text{joint})})$ can be given as follows:

$$
\begin{aligned}
& u(\boldsymbol{X}^{(\text{joint})}) \\
&= \sum_{i=1}^{\infty}\sqrt{\xi_i}u_i^*\phi_i(\boldsymbol{X}^{(\text{joint})}) \\
&= \mathbb{E}_{r(\boldsymbol{X}^{(\text{joint})})}
\begin{bmatrix}
-\lambda\nabla_{\tilde{\boldsymbol{X}}^{(\text{miss})}}K(\boldsymbol{X}^{(\text{joint})}, \tilde{\boldsymbol{X}}^{(\text{joint})}) \\
+ \nabla_{\tilde{\boldsymbol{X}}^{(\text{miss})}}\log\hat{p}(\tilde{\boldsymbol{X}}^{(\text{joint})})K(\boldsymbol{X}^{(\text{miss})}, \tilde{\boldsymbol{X}}^{(\text{miss})})
\end{bmatrix}.
\end{aligned} \tag{C.34}
$$

**Lower Bound Acquirement:**

Before starting the proving of this part, we should notice that given the unchanged observational data $\boldsymbol{X}^{(\text{obs})}$, the distribution $p(\boldsymbol{X}^{(\text{obs})})$ is a constant. On this basis, consider the definition of $\mathcal{F}_{\text{NER}}$ (right-hand-side of Eq. (7)), the first term and the second term are denoted by 'term 1' and 'term 2' for simplicity:

$$
\underbrace{\mathbb{E}_{r(\boldsymbol{X}^{(\text{miss})})}[\log\hat{p}(\boldsymbol{X}^{(\text{miss})}|\boldsymbol{X}^{(\text{obs})})]}_{:=\text{term 1}} + \lambda \times \left[\underbrace{-\mathbb{H}[r(\boldsymbol{X}^{(\text{miss})})]}_{:=\text{term 2}}\right]. \tag{C.35}
$$

For term 1, we can obtain the following derivation:

$$
\int r(\boldsymbol{X}^{(\text{miss})}) \log \hat{p}(\boldsymbol{X}^{(\text{miss})}|\boldsymbol{X}^{(\text{obs})}) \mathrm{d}\boldsymbol{X}^{(\text{miss})}
$$

$$
\geq \int r(\boldsymbol{X}^{(\text{miss})}) \log \hat{p}(\boldsymbol{X}^{(\text{miss})}|\boldsymbol{X}^{(\text{obs})}) \mathrm{d}\boldsymbol{X}^{(\text{miss})} + \underbrace{\int p(\boldsymbol{X}^{(\text{obs})}) \log p(\boldsymbol{X}^{(\text{obs})}) \mathrm{d}\boldsymbol{X}^{(\text{obs})}}_{\text{negative entropy (negative constant)}}
$$

$$
= \iint p(\boldsymbol{X}^{(\text{obs})}) r(\boldsymbol{X}^{(\text{miss})}) \log \hat{p}(\boldsymbol{X}^{(\text{miss})}|\boldsymbol{X}^{(\text{obs})}) \mathrm{d}\boldsymbol{X}^{(\text{miss})} \mathrm{d}\boldsymbol{X}^{(\text{obs})}
$$

$$
+ \underbrace{\int p(\boldsymbol{X}^{(\text{obs})}) \log p(\boldsymbol{X}^{(\text{obs})}) \mathrm{d}\boldsymbol{X}^{(\text{obs})}}_{\text{negative constant}}
$$

$$
= \iint p(\boldsymbol{X}^{(\text{obs})}) r(\boldsymbol{X}^{(\text{miss})}) \log \hat{p}(\boldsymbol{X}^{(\text{miss})}|\boldsymbol{X}^{(\text{obs})}) \mathrm{d}\boldsymbol{X}^{(\text{miss})} \mathrm{d}\boldsymbol{X}^{(\text{obs})}
\tag{C.36}
$$

$$
+ \underbrace{\iint r(\boldsymbol{X}^{(\text{miss})}) p(\boldsymbol{X}^{(\text{obs})}) \log p(\boldsymbol{X}^{(\text{obs})}) \mathrm{d}\boldsymbol{X}^{(\text{miss})} \mathrm{d}\boldsymbol{X}^{(\text{obs})}}_{\text{negative constant}}
$$

$$
= \iint \underbrace{p(\boldsymbol{X}^{(\text{obs})}) r(\boldsymbol{X}^{(\text{miss})})}_{r(\boldsymbol{X}^{(\text{miss})}, \boldsymbol{X}^{(\text{obs})})} \underbrace{[\log \hat{p}(\boldsymbol{X}^{(\text{miss})}|\boldsymbol{X}^{(\text{obs})}) + \log p(\boldsymbol{X}^{(\text{obs})})]}_{\log \hat{p}(\boldsymbol{X}^{(\text{miss})}, \boldsymbol{X}^{(\text{obs})})} \mathrm{d}\boldsymbol{X}^{(\text{miss})} \mathrm{d}\boldsymbol{X}^{(\text{obs})}
$$

$$
= \mathbb{E}_{r(\boldsymbol{X}^{(\text{miss})}, \boldsymbol{X}^{(\text{obs})})} [\log \hat{p}(\boldsymbol{X}^{(\text{miss})}, \boldsymbol{X}^{(\text{obs})})].
$$

Similarly, the term 2 can be reformulated as follows:

$$
- \mathbb{H}[r(\boldsymbol{X}^{(\text{miss})})]
$$

$$
\geq - \mathbb{H}[r(\boldsymbol{X}^{(\text{miss})})] + \underbrace{\int p(\boldsymbol{X}^{(\text{obs})}) \log p(\boldsymbol{X}^{(\text{obs})}) \mathrm{d}\boldsymbol{X}^{(\text{obs})}}_{\text{negative entropy (negative constant)}}
$$

$$
= \iint p(\boldsymbol{X}^{(\text{obs})}) r(\boldsymbol{X}^{(\text{miss})}) \log r(\boldsymbol{X}^{(\text{miss})}) \mathrm{d}\boldsymbol{X}^{(\text{miss})} \mathrm{d}\boldsymbol{X}^{(\text{obs})}
$$

$$
+ \underbrace{\iint p(\boldsymbol{X}^{(\text{obs})}) r(\boldsymbol{X}^{(\text{miss})}) \log p(\boldsymbol{X}^{(\text{obs})}) \mathrm{d}\boldsymbol{X}^{(\text{miss})} \mathrm{d}\boldsymbol{X}^{(\text{obs})}}_{\text{negative entropy (negative constant)}}
\tag{C.37}
$$

$$
= \iint \underbrace{p(\boldsymbol{X}^{(\text{obs})}) r(\boldsymbol{X}^{(\text{miss})})}_{r(\boldsymbol{X}^{(\text{obs})}, \boldsymbol{X}^{(\text{miss})})} \underbrace{[\log r(\boldsymbol{X}^{(\text{miss})}) + \log p(\boldsymbol{X}^{(\text{obs})})]}_{r(\boldsymbol{X}^{(\text{obs})}, \boldsymbol{X}^{(\text{miss})})} \mathrm{d}\boldsymbol{X}^{(\text{miss})} \mathrm{d}\boldsymbol{X}^{(\text{obs})}
$$

$$
= - \mathbb{H}[r(\boldsymbol{X}^{(\text{obs})}, \boldsymbol{X}^{(\text{miss})})].
$$

Combine Eqs. (C.36) and (C.37), we can obtain the following relationship:

$$
\mathcal{F}_{\text{NER}} - \text{const} = \mathcal{F}_{\text{joint-NER}},
\tag{C.38}
$$

and constant const is greater than 0.

$\square$

**Corollary** (4.3). *The following equation holds:* $u(\boldsymbol{X}^{(joint)}) = u(\boldsymbol{X}^{(miss)})$.

*Proof.* This corollary can be easily proven by according to Eq. (C.38):

$$
\mathcal{F}_{\text{NER}} = \mathcal{F}_{\text{joint-NER}} + \text{const}
$$

$$
\Rightarrow \nabla_{\boldsymbol{X}^{(\text{miss})}} \frac{\delta \mathcal{F}_{\text{NER}}}{\delta r(\boldsymbol{X}^{(\text{miss})})} = \nabla_{\boldsymbol{X}^{(\text{miss})}} \frac{\delta \mathcal{F}_{\text{joint-NER}} + \text{const}}{\delta r(\boldsymbol{X}^{(\text{miss})})}
\tag{C.39}
$$

$$
\Rightarrow \nabla_{\boldsymbol{X}^{(\text{miss})}} \frac{\delta \mathcal{F}_{\text{NER}}}{\delta r(\boldsymbol{X}^{(\text{miss})})} = \nabla_{\boldsymbol{X}^{(\text{miss})}} \frac{\delta \mathcal{F}_{\text{joint-NER}}}{\delta r(\boldsymbol{X}^{(\text{miss})})}.
$$

Plugging Eq. (C.39) into Eqs. (A.5) and (A.6), we can see that the velocity fields for $X^{(\text{miss})}$ within functional $\mathcal{F}_{\text{NER}}$ and $\mathcal{F}_{\text{joint-NER}}$ are identical. $\qquad\qquad\qquad\qquad\square$

## Appendix D Detailed Explanation for the Workflow of NewImp Approach

In this section, we intend to provide detailed information about the implementation of the NewImp approach in Algorithm 1. We will focus on two primary aspects: 1) the numerical implementation of ODE simulation, and 2) the DSM algorithm.

### D.1 Forward Euler's Method for ODE Simulation

During step 7 of Algorithm 1, we involve the simulation of the ODE defined by Eqs. (9) and (14). To simulate this ODE we use the forward Euler's method [6] in this paper for simplicity. Specifically, suppose we have the following ODE:

$$\frac{\mathrm{d}x_\tau}{\mathrm{d}\tau} = f(x_\tau), \tag{D.1}$$

and the initial value at $\tau = 0$ is given $x_0 = x_{\text{init}}$, the value at time $\eta$ can be derived as follows:

$$x_\eta = x_0 + \int_0^\eta f(x_\tau)\mathrm{d}\tau. \tag{D.2}$$

To alleviate the intergal term, the forward Euler's method attempts to approximate the integral term to summation term as follows:

$$x_\eta \approx x_0 + f(x_\tau) \times (\eta - 0). \tag{D.3}$$

On this basis, the value at time T can be obtained by repeating Eq. (D.3) from $\tau = 0$ to $\tau = \text{T}$, which is the forward Euler's method.

---

**Algorithm 2** Algorithm for Forward Euler's Method

1: **Input:** ODE $f(x_\tau)$; start point 0; end point T; step size $\eta$; initial value $x_0$.
2: **Output:** Predicted value $x_{\text{T}}$ at $\tau = \text{T}$.
3: $j \leftarrow \frac{\text{T}-0}{\eta}$ $\qquad\qquad\qquad\qquad\qquad\qquad\qquad\qquad$ ▷ Calculate the Number of Steps
4: **for** $\tau = 0 + \eta, 0 + 2\eta, \ldots, 0 + j\eta$ **do**
5: $\quad x_\tau \leftarrow x_{\tau-\eta} + f(x_{\tau-\eta}) \times \eta$
6: **end for**

---

### D.2 Detailed Information for DSM

During step 5 of Algorithm 1, we involve the training of $\nabla_{\boldsymbol{X}^{(\text{joint})}} \log \hat{p}(\boldsymbol{X}^{(\text{joint})})$ using the DSM function. In this subsection, we aim to further elaborate on the detailed algorithm for the DSM function to uphold the completeness of this manuscript. As mentioned in Section 2.2, the score function $\nabla_{\boldsymbol{X}^{(\text{joint})}} \log \hat{p}(\boldsymbol{X}^{(\text{joint})})$ is typically parameterized by a neural network. For simplicity, we denote the parameter set of $\nabla_{\boldsymbol{X}^{(\text{joint})}} \log \hat{p}(\boldsymbol{X}^{(\text{joint})})$ by $\theta$.

---

**Algorithm 3** DSM for $\nabla_{\boldsymbol{X}^{(\text{joint})}} \log \hat{p}(\boldsymbol{X}^{(\text{joint})})$ Training

---

1: **Input:** joint data $\boldsymbol{X}^{(\text{joint})}$.
2: **Hyperparameters:** neural network learning rate $lr$, training epoch $\mathcal{E}$, and neural network hidden unit $\text{HU}_{\text{score}}$.
3: **for** $e = 1$ to $\mathcal{E}$ **do**
4:    $\hat{\boldsymbol{X}}^{(\text{joint})} \leftarrow \boldsymbol{X}^{(\text{joint})} + \epsilon, \epsilon \sim \mathcal{N}(\boldsymbol{0}, \sigma^2 \mathbf{I})$                           ▷ Data Noising
5:    $\nabla_{\hat{\boldsymbol{X}}^{(\text{joint})}} \log q_\sigma(\hat{\boldsymbol{X}}^{(\text{joint})} | \boldsymbol{X}^{(\text{joint})}) \leftarrow -\dfrac{\hat{\boldsymbol{X}}^{(\text{joint})} - \boldsymbol{X}^{(\text{joint})}}{\sigma^2}$
6:    $\mathcal{L}_{\text{DSM}} \leftarrow$ Eq. (15)
7:    $\theta \leftarrow \texttt{ApplyGradient}(\nabla_\theta \mathcal{L}_{\text{DSM}}, lr)$         ▷ Apply the Gradient with Learning Rate $lr$
8: **end for**

---

## Appendix E    Detailed Information for Experiments

### E.1    Background & Simulation of Missing Data

Table E.1: Detailed dataset descriptions, where 'Dimension' denotes the variate number of each dataset. 'Numer' denotes the total number of item.

| Abbreviation | Dataset Name | Numer (N) | Dimension (D) |
|---|---|---|---|
| BT | Blood Transfusion | 748 | 4 |
| BCD | Breast Cancer Diagnostic | 569 | 30 |
| CC | Concrete Compression | 1030 | 7 |
| CBV | Connectionist Bench Vowel | 990 | 10 |
| IS | Ionosphere | 351 | 34 |
| PK | Parkinsons | 195 | 23 |
| QB | QSAR Biodegradation | 1055 | 41 |
| WQW | Wine Quality White | 4898 | 11 |

In this paper, we consider the datasets listed in Table E.1 as our experimental datasets. Based on this, according to reference [44], missing data can be classified into three categories: Missing at Random (MAR), where the likelihood of missing data depends solely on observed data; Missing Completely at Random (MCAR), where the absence of data is completely unrelated to any observed or unobserved variables; and Missing Not at Random (MNAR), where missingness is influenced by unobserved data. In the cases of MCAR and MAR, the patterns of missing data are considered 'ignorable' because it is unnecessary to explicitly model the distribution of the missing values. Conversely, MNAR scenarios, where missing data can introduce significant biases that are not easily corrected without imposing domain-specific assumptions, constraints, or parametric forms on the missingness mechanism, present more complex challenges [40, 22]. Therefore, our discussion is primarily focused on numerical tabular data within the MCAR and MAR contexts.

To simulate missing data, we adopt the methodologies outlined in reference [22]:

- **MAR:** Initially, a random subset of features is selected to remain non-missing. The masking of the remaining features is conducted using a logistic model, which employs the non-missing features as predictors. This model is parameterized with randomly selected weights, and the bias is adjusted to achieve the desired missingness rate.

- **MCAR:** For each data point, the masking variable is generated from a Bernoulli distribution with a predetermined fixed mean, ensuring that the probability of missingness is the same across all data points.

- **MNAR:** Although MNAR scenarios are not the primary focus of this manuscript, we include experiments in this context. Missingness is introduced either by additional masking of the MAR-selected features using a Bernoulli process with a fixed mean, or through direct self-masking of values using interval-censoring techniques. In this paper, we mainly consider the former strategy. In other words, the mechanism of MNAR we used in this paper is identical to the previously described MAR mechanism, but the inputs of the logistic model are then masked by an MCAR mechanism.

## E.2 Hyperparameter Setting of Baseline Models

In this subsection, we want to report the baseline models' hyperparameter settings to ensure the reproducibility of our paper:

- **Batch-Size-Related:** The batch size for ReMasker is set to 64. For other baseline models, it is uniformly set at 512. (Notably, for Sink and TDM, if $N < 512$, the batch size is set to $2^{\lfloor \frac{N}{2} \rfloor}$. )

- **Hidden-Unit-Related:**
    - The MIWAE model features a latent dimension of 16 and 32 hidden units.
    - The TDM model includes 16 hidden units per layer with the number of layers set to 2.
    - MIRACLE's hidden units are set to 32.
    - For ReMasker, the embedding dimension is 32, depth is 6, mask ratio is 0.5, encoder depth is 6, decoder depth is 4, number of heads is 4, and the multi-layer perceptron ratio is 4.0.
    - For MissDiff and CSDI_T, the channel size is set as 16, the embedding dimension is set to 128, and the layer number is set as 2.
    - For the GAIN model, for both the generator and the discriminator, the hidden size is set to $2 \times D$, and the number of hidden layers is set to 3.

- **Diffusion-Hyperparameters-Related:** The diffusion step is set at 100 and the particle number at 50 for MissDiff and CSDI_T.

# Appendix F   Additional Empirical Evidence

## F.1   Toy Case Experiments

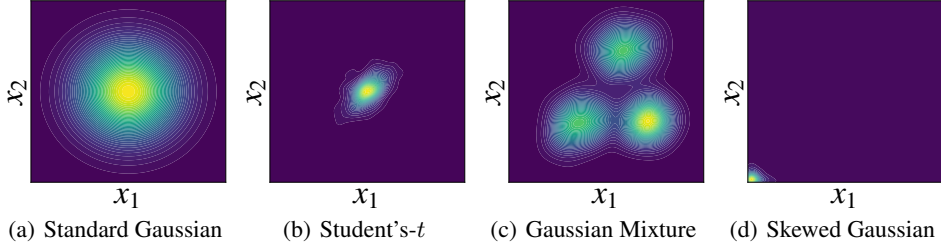

| | (a) Standard Gaussian | (b) Student's-$t$ | (c) Gaussian Mixture | (d) Skewed Gaussian |
|---|---|---|---|---|

Figure F.1: Contours of Various Distributions' Density Value.

Table F.1: NewImp Performance with Missing Rate at 30%, and 1000 samples are generated.

| Scenario | Distribution Type | MAE | WASS |
|---|---|---|---|
| MAR | Gaussian | $0.769_{\pm 0.030}$ | $0.481_{\pm 0.026}$ |
| | Student's-$t$ | $0.737_{\pm 0.053}$ | $0.513_{\pm 0.048}$ |
| | Gaussian Mixture | $0.763_{\pm 0.097}$ | $0.419_{\pm 0.104}$ |
| | Skewed-Gaussian | $0.422_{\pm 0.253}$ | $0.492_{\pm 0.025}$ |
| MCAR | Gaussian | $0.769_{\pm 0.013}$ | $0.287_{\pm 0.014}$ |
| | Student's-$t$ | $0.698_{\pm 0.030}$ | $0.307_{\pm 0.014}$ |
| | Gaussian Mixture | $0.824_{\pm 0.017}$ | $0.391_{\pm 0.023}$ |
| | Skewed-Gaussian | $0.417_{\pm 0.140}$ | $0.210_{\pm 0.026}$ |
| MNAR | Gaussian | $0.778_{\pm 0.034}$ | $0.309_{\pm 0.030}$ |
| | Student's-$t$ | $0.715_{\pm 0.028}$ | $0.323_{\pm 0.019}$ |
| | Gaussian Mixture | $0.807_{\pm 0.042}$ | $0.380_{\pm 0.050}$ |
| | Skewed-Gaussian | $0.421_{\pm 0.111}$ | $0.202_{\pm 0.006}$ |

To demonstrate the effectiveness of the NewImp method vary different type of distributions, we evaluate it across four distinct toy cases, each characterized by different distributions:

- **Standard Gaussian:** $\boldsymbol{X}^{(\text{ideal})} \sim \mathcal{N}(0, I_{2\times 2})$.

- **Student's-$t$ (a heavy-tailed distribution):** $\boldsymbol{X}^{(\text{ideal})} \sim \text{St-}t(0, \begin{bmatrix} 1 & 0.5 \\ 0.5 & 1 \end{bmatrix})$.

- **Gaussian Mixture:** $\boldsymbol{X}^{(\text{ideal})} \sim \frac{1}{3} \times \mathcal{N}([1,2], \begin{bmatrix} 0.5 & 0 \\ 0 & 0.5 \end{bmatrix}) + \frac{1}{3} \times \mathcal{N}([-1,-2], \begin{bmatrix} 0.5 & 0.1 \\ 0.1 & 0.5 \end{bmatrix}) + \frac{1}{3} \times \mathcal{N}([2,-2], \begin{bmatrix} 0.3 & 0 \\ 0 & 0.3 \end{bmatrix})$.

- **Skewed Gaussian (via exponential transformation):** $\boldsymbol{X}^{(\text{ideal})} = \exp(\epsilon)$, where $\epsilon \sim \mathcal{N}(0, I_{2\times 2})$.

Based on this, we display the contours of their density values in Fig. F.1, and we list the imputation accuracy comparisons for MAR, MCAR, and MNAR scenarios with a 30% missing rate in Table F.1. The results indicate that our NewImp approach generally performs better on non-standard Gaussian type data, underscoring its universality and applicability. This enhanced performance is attributable to our modeling strategy, which involves modeling the score function of the data [52, 50], which eliminates the need for normalization, and consequently results in the NewImp approach can perform well on complex data distributions, including skewed, heavy-tailed, and mixture distributions.

## F.2   Additional Experimental Results with MNAR Scenario

In this section, we expand upon the results presented in Table 1 by including the MNAR scenario, as detailed in Table F.2. Additionally, we report on the outcomes of an ablation study and sensi-

tivity analysis in Tables F.4 and F.5 and Fig. F.2. These extended results lead to several pertinent observations:

- Across three different missing data scenarios, the models consistently exhibit the poorest performance under the MNAR condition. For instance, in the MNAR scenario, nearly all models show a significant decrease in imputation accuracy and an increase in standard deviation. This supports the assertion made in Appendix E.1 that addressing the MNAR scenario requires the incorporation of relevant domain knowledge to mitigate biases introduced by the pattern of missing data.
- The findings from the ablation study under the MNAR scenario are consistent with those observed in both MAR and MCAR scenarios in Section 5.3. This consistency underscores the importance of including the NER term and adopting the joint distribution modeling approach.
- Similarly, the results from the sensitivity analysis under the MNAR scenario align with those from MAR and MCAR scenarios in Section 5.4. This alignment reinforces our interpretations of model performance across different groups of hyperparameters under MAR and MCAR scenarios.

Table F.2: Performance of MAE and WASS metrics at 30% missing rate.

| Scenario | Model | BT MAE | BT WASS | BCD MAE | BCD WASS | CC MAE | CC WASS | CBV MAE | CBV WASS | IS MAE | IS WASS | PK MAE | PK WASS | QB MAE | QB WASS | WQW MAE | WQW WASS |
|---|---|---|---|---|---|---|---|---|---|---|---|---|---|---|---|---|---|
| MAR | CSDI_T | 0.93 * | 3.44 * | 0.92 * | 18.20 * | 0.85 * | 2.82 * | 0.81 * | 3.86 * | 0.70 * | 16.86 * | 0.99 * | 15.86 * | 0.65 * | 20.10 * | 0.77 * | 4.13 * |
| | MissDiff | 0.85 * | 2.20 * | 0.91 * | 16.53 * | 0.87 * | 1.59 * | 0.83 * | 3.87 * | 0.72 * | 13.25 * | 0.92 * | 17.07 * | 0.63 * | 26.25 * | 0.75 * | 6.88 * |
| | GAIN | 0.75 * | 0.65 * | 0.54 * | 1.64 * | 0.75 * | 0.67 * | 0.68 * | 0.68 * | 0.56 * | 1.88 * | 0.59 * | 1.90 * | 0.65 * | 5.05 * | 0.68 * | 0.87 * |
| | MIRACLE | 0.62 * | 0.38 | 0.55 * | 1.92 * | 0.43 | 0.25 | 0.55 * | 0.46 * | 3.39 * | 35.06 * | 4.14 * | 34.07 * | 0.46 | 2.87 * | 0.51 * | 0.56 |
| | MIWAE | 0.64 | 0.53 | 0.52 * | 1.54 * | 0.76 * | 0.64 * | 0.82 * | 0.92 * | 0.50 * | 1.87 * | 0.65 * | 1.98 * | 0.55 * | 5.05 * | 0.62 * | 0.75 * |
| | Sink | 0.87 * | 0.92 * | 0.92 * | 3.84 * | 0.88 * | 0.83 * | 0.84 * | 0.98 * | 0.75 * | 2.43 * | 0.94 * | 3.61 * | 0.65 * | 4.71 * | 0.76 * | 1.04 * |
| | TDM | 0.83 * | 0.89 * | 0.83 * | 3.47 * | 0.81 * | 0.73 * | 0.76 * | 0.85 * | 0.62 * | 1.96 * | 0.86 * | 3.36 * | 0.59 * | 4.46 * | 0.73 * | 0.99 * |
| | ReMasker | **0.52** | 0.52 | 0.48 * | 1.15 | 0.60 * | 0.43 * | **0.49** * | **0.37** * | 0.62 * | 2.23 * | 0.61 * | 1.59 * | 0.60 * | 3.81 | 0.51 * | 0.59 * |
| | **NewImp** | 0.52 | **0.38** | **0.34** | **0.82** | **0.35** | 0.25 | **0.31** | **0.20** | **0.39** | **1.31** | **0.44** | **1.21** | **0.45** | 3.50 | **0.46** | **0.55** |
| MCAR | CSDI_T | 0.73 * | 1.93 * | 0.73 * | 15.51 * | 0.85 * | 2.71 * | 0.83 * | 3.79 * | 0.76 * | 15.19 * | 0.72 * | 12.42 * | 0.57 * | 19.89 * | 0.78 * | 4.11 * |
| | MissDiff | 0.72 * | 1.62 * | 0.73 * | 14.39 * | 0.84 * | 1.23 * | 0.82 * | 3.31 * | 0.75 * | 13.01 * | 0.71 * | 14.12 * | 0.56 * | 19.67 * | 0.76 * | 4.95 * |
| | GAIN | 0.72 * | 0.39 * | **0.38** * | 1.41 * | 0.78 * | 0.73 * | 0.72 * | 0.99 * | 0.57 * | 3.72 * | 0.46 * | 1.70 | 0.42 * | 3.62 | 0.73 * | 1.14 * |
| | MIRACLE | 0.52 | 0.15 * | 0.44 * | 1.94 * | 0.53 * | 0.35 | 0.61 * | 0.72 * | 2.99 * | 52.92 * | 3.38 * | 42.78 * | 0.35 | **2.71** * | 0.56 * | **0.75** |
| | MIWAE | 0.58 * | 0.24 | 0.50 * | 2.55 * | 0.76 * | 0.69 * | 0.83 * | 1.24 * | 0.64 * | 4.95 * | 0.51 * | 2.05 * | 0.48 * | 5.87 * | 0.67 * | 0.95 * |
| | Sink | 0.73 * | 0.48 * | 0.75 * | 4.39 * | 0.84 * | 0.85 * | 0.82 * | 1.27 * | 0.75 * | 4.94 * | 0.74 * | 3.36 * | 0.61 * | 5.92 * | 0.76 * | 1.25 * |
| | TDM | 0.68 * | 0.42 * | 0.63 * | 3.57 * | 0.77 * | 0.75 * | 0.77 * | 1.15 * | 0.66 * | 4.20 * | 0.64 * | 2.89 * | 0.52 * | 5.34 * | 0.74 * | 1.20 * |
| | ReMasker | **0.46** * | **0.11** | 0.39 * | 1.69 * | 0.55 * | 0.37 | 0.56 * | 0.64 * | 0.54 * | 4.01 * | 0.48 * | 1.71 * | 0.45 * | 3.94 | 0.57 * | 0.76 |
| | **NewImp** | 0.48 | 0.18 | **0.25** | **0.80** | **0.47** | **0.34** | **0.42** | 0.44 | **0.44** | 3.05 | **0.32** | **1.01** | **0.34** | 3.66 | **0.53** | 0.76 |
| MNAR | CSDI_T | 0.83 * | 2.29 * | 0.82 * | 15.68 * | 0.85 * | 2.78 * | 0.83 * | 3.83 * | 0.74 * | 15.54 * | 0.84 * | 12.20 * | 0.62 * | 19.77 * | 0.78 * | 4.09 * |
| | MissDiff | 0.78 * | 1.43 * | 0.81 * | 14.89 * | 0.84 * | 1.27 * | 0.83 * | 3.53 * | 0.72 * | 13.31 * | 0.81 * | 16.02 * | 0.61 * | 21.62 * | 0.76 * | 4.70 * |
| | GAIN | 0.77 * | 0.57 * | 0.62 * | 3.94 * | 0.78 * | 0.79 * | 0.78 * | 1.15 * | 0.71 * | 4.85 * | 0.70 * | 4.20 * | 0.76 * | 10.53 * | 0.75 * | 1.23 * |
| | MIRACLE | 0.63 | 0.35 | 0.60 * | 4.26 * | 0.52 * | 0.35 | 0.63 * | 0.77 * | 3.10 * | 55.56 * | 3.49 * | 44.76 * | 0.52 * | 5.61 | 0.58 * | 0.80 |
| | MIWAE | 0.66 * | 0.42 | 0.56 * | 3.31 * | 0.74 * | 0.68 * | 0.85 * | 1.30 * | 0.59 * | 4.33 * | 0.60 * | 3.06 * | 0.53 * | 7.21 * | 0.67 * | 0.97 * |
| | Sink | 0.79 * | 0.68 * | 0.83 * | 5.90 * | 0.83 * | 0.89 * | 0.84 * | 1.36 * | 0.75 * | 4.86 * | 0.84 * | 5.02 * | 0.64 * | 7.23 * | 0.77 * | 1.33 * |
| | TDM | 0.76 * | 0.64 * | 0.74 * | 5.18 * | 0.76 * | 0.77 * | 0.79 * | 1.24 * | 0.64 * | 4.02 * | 0.76 * | 4.54 * | 0.57 * | 6.45 | 0.74 * | 1.23 * |
| | ReMasker | **0.53** | **0.28** | 0.42 * | 1.91 * | 0.54 * | **0.39** * | 0.59 * | 0.68 * | 0.51 * | 3.59 * | 0.63 * | 3.06 * | 0.47 | 5.02 | 0.56 | **0.77** |
| | **NewImp** | 0.60 | 0.35 | **0.32** | **1.46** | **0.44** | 0.34 | **0.46** | **0.52** | **0.40** | 2.68 | **0.39** | **1.56** | **0.42** | 5.57 | **0.55** | 0.81 |

*Kindly Note*: The best results are **bolded** and the second best results are underliend. "*" marks the results that NewImp significantly outperform with $p$-value$< 0.05$ over paired samples $t$-test.

Table F.3: Standard deviation of MAE and WASS metrics at 30% missing rate.

| Scenario | Model | BT MAE | BT WASS | BCD MAE | BCD WASS | CC MAE | CC WASS | CBV MAE | CBV WASS | IS MAE | IS WASS | PK MAE | PK WASS | QB MAE | QB WASS | WQW MAE | WQW WASS |
|---|---|---|---|---|---|---|---|---|---|---|---|---|---|---|---|---|---|
| MAR | CSDI_T | 4.8E-2 | 2.9E-1 | 5.9E-2 | 2.6E0 | 2.7E-2 | 1.4E-1 | 2.0E-2 | 9.2E-2 | 2.1E-2 | 2.0E0 | 3.9E-2 | 3.1E0 | 2.0E-2 | 8.3E-1 | 1.8E-2 | 7.7E-2 |
| | MissDiff | 4.0E-2 | 4.9E-1 | 3.1E-2 | 2.6E0 | 3.4E-2 | 2.6E-1 | 1.9E-2 | 1.2E0 | 5.4E-2 | 5.6E-1 | 3.0E-2 | 2.4E0 | 1.8E-2 | 4.9E0 | 1.6E-2 | 1.3E0 |
| | GAIN | 1.0E-1 | 1.6E-1 | 4.3E-2 | 2.3E-1 | 3.4E-2 | 8.9E-2 | 1.9E-2 | 4.0E-2 | 5.8E-2 | 3.4E-1 | 5.4E-2 | 3.7E-1 | 6.9E-2 | 8.4E-1 | 3.9E-2 | 5.6E-2 |
| | MIRACLE | 2.0E-2 | 6.3E-2 | 5.1E-2 | 4.0E-1 | 1.1E-2 | 2.0E-2 | 1.8E-2 | 2.1E-2 | 6.7E-1 | 1.2E1 | 4.2E-1 | 5.6E0 | 1.6E-2 | 1.9E-1 | 1.3E-2 | 2.8E-2 |
| | MIWAE | 6.5E-2 | 1.5E-1 | 5.2E-2 | 2.5E-1 | 6.1E-2 | 1.2E-1 | 2.4E-2 | 4.6E-2 | 5.3E-2 | 1.8E-1 | 2.7E-2 | 1.8E-1 | 3.8E-2 | 2.8E-1 | 1.9E-2 | 2.7E-2 |
| | Sink | 4.6E-2 | 1.2E-1 | 3.2E-2 | 1.6E-1 | 2.6E-2 | 7.4E-2 | 2.4E-2 | 5.5E-2 | 5.0E-2 | 2.0E-1 | 1.7E-2 | 9.3E-2 | 1.7E-2 | 7.6E-2 | 2.2E-2 | 4.4E-2 |
| | TDM | 4.5E-2 | 1.2E-1 | 1.9E-2 | 8.2E-2 | 3.4E-2 | 8.6E-2 | 2.6E-2 | 5.2E-2 | 6.2E-2 | 2.1E-1 | 2.7E-2 | 1.7E-1 | 1.1E-2 | 8.1E-2 | 2.1E-2 | 4.7E-2 |
| | ReMasker | 1.1E-1 | 2.4E-1 | 6.1E-2 | 2.0E-1 | 7.5E-2 | 8.4E-2 | 2.6E-2 | 3.0E-2 | 9.2E-2 | 4.8E-1 | 5.2E-2 | 1.8E-1 | 2.3E-2 | 3.6E-1 | 1.5E-2 | 2.2E-2 |
| | **NewImp** | 2.0E-2 | 4.0E-2 | 2.7E-2 | 1.1E-1 | 5.6E-2 | 6.4E-2 | 1.6E-2 | 2.2E-2 | 1.9E-2 | 1.1E-1 | 1.1E-2 | 8.8E-2 | 1.9E-2 | 2.7E-1 | 1.6E-2 | 3.3E-2 |
| MCAR | CSDI_T | 1.0E-2 | 1.5E-1 | 8.7E-3 | 5.7E-1 | 8.7E-3 | 8.2E-2 | 4.6E-3 | 4.6E-2 | 4.6E-3 | 3.6E-1 | 1.1E-2 | 8.7E-1 | 3.7E-3 | 3.1E-1 | 1.2E-2 | 5.0E-2 |
| | MissDiff | 6.4E-3 | 3.3E-1 | 8.2E-3 | 8.3E-1 | 3.5E-3 | 2.3E-1 | 5.9E-3 | 8.4E-1 | 7.1E-3 | 1.8E-1 | 4.6E-3 | 2.5E0 | 6.2E-3 | 2.4E0 | 4.1E-3 | 6.5E-1 |
| | GAIN | 6.1E-2 | 1.0E-1 | 7.9E-3 | 2.6E-2 | 2.4E-2 | 5.2E-2 | 1.4E-2 | 3.4E-2 | 1.8E-2 | 1.8E-1 | 4.5E-2 | 3.8E-1 | 3.7E-3 | 1.8E-1 | 2.0E-2 | 5.5E-2 |
| | MIRACLE | 2.6E-2 | 8.4E-3 | 1.6E-2 | 1.9E-1 | 1.7E-2 | 1.5E-2 | 5.2E-3 | 1.4E-2 | 4.3E-2 | 1.2E0 | 4.6E-2 | 1.1E0 | 1.0E-2 | 1.7E-1 | 1.1E-3 | 5.5E-3 |
| | MIWAE | 3.1E-2 | 3.9E-2 | 4.8E-3 | 4.9E-2 | 7.6E-3 | 1.3E-2 | 1.6E-2 | 4.3E-2 | 9.4E-3 | 1.3E-1 | 1.0E-2 | 7.8E-2 | 9.1E-3 | 2.7E-1 | 4.1E-3 | 9.5E-3 |
| | Sink | 7.3E-3 | 3.4E-2 | 4.6E-3 | 2.5E-2 | 7.0E-3 | 6.6E-3 | 4.5E-3 | 4.5E-3 | 4.2E-3 | 1.4E-1 | 1.0E-2 | 5.9E-2 | 3.3E-3 | 1.9E-1 | 3.9E-3 | 1.2E-2 |
| | TDM | 4.9E-3 | 2.8E-2 | 8.7E-3 | 3.1E-2 | 9.8E-3 | 6.6E-3 | 6.9E-3 | 7.9E-3 | 1.0E-3 | 1.9E-3 | 3.3E-3 | 3.6E-2 | 9.3E-3 | 1.5E-1 | 5.1E-3 | 1.3E-2 |
| | ReMasker | 1.5E-3 | 1.7E-2 | 2.0E-2 | 9.7E-2 | 3.2E-2 | 2.9E-2 | 1.7E-2 | 3.1E-2 | 9.3E-3 | 1.4E-1 | 3.8E-2 | 3.1E-1 | 4.8E-2 | 3.9E-1 | 7.6E-3 | 1.7E-2 |
| | **NewImp** | 3.3E-3 | 3.7E-3 | 1.9E-3 | 4.6E-2 | 1.1E-2 | 1.8E-2 | 4.1E-3 | 1.8E-2 | 5.7E-3 | 1.1E-1 | 6.4E-3 | 3.7E-2 | 4.8E-3 | 1.7E-1 | 2.2E-3 | 1.1E-2 |
| MNAR | CSDI_T | 2.9E-2 | 2.2E-1 | 8.7E-3 | 7.8E-1 | 2.2E-2 | 1.3E-1 | 7.4E-3 | 7.4E-2 | 1.0E-2 | 5.9E-1 | 2.2E-2 | 1.8E0 | 2.6E-3 | 4.6E-1 | 2.4E-3 | 2.5E-2 |
| | MissDiff | 3.7E-2 | 3.7E-1 | 2.4E-3 | 9.7E-1 | 5.9E-3 | 2.4E-1 | 5.5E-3 | 8.2E-1 | 1.4E-2 | 3.3E-1 | 1.0E-2 | 2.1E0 | 8.7E-3 | 3.3E0 | 4.0E-3 | 5.2E-1 |
| | GAIN | 4.9E-2 | 1.2E-1 | 6.2E-2 | 6.9E-1 | 5.3E-2 | 8.6E-2 | 4.1E-2 | 9.3E-2 | 5.5E-3 | 4.8E-2 | 2.5E-2 | 4.7E-1 | 5.0E-2 | 1.2E0 | 4.0E-2 | 1.0E-1 |
| | MIRACLE | 6.6E-2 | 9.5E-2 | 1.9E-2 | 4.7E-1 | 1.3E-2 | 1.3E-2 | 4.0E-3 | 1.7E-2 | 9.9E-2 | 3.5E0 | 6.9E-2 | 1.6E0 | 1.7E-2 | 1.7E-1 | 7.5E-3 | 1.2E-2 |
| | MIWAE | 3.3E-2 | 6.4E-2 | 8.3E-3 | 3.7E-2 | 2.4E-2 | 3.5E-2 | 3.0E-2 | 8.7E-2 | 6.6E-3 | 7.2E-2 | 2.3E-2 | 3.2E-1 | 1.2E-2 | 1.5E-1 | 9.1E-3 | 2.0E-2 |
| | Sink | 1.9E-2 | 6.2E-2 | 1.4E-2 | 1.4E-1 | 1.0E-2 | 6.3E-3 | 1.3E-2 | 3.9E-2 | 7.2E-3 | 5.1E-2 | 1.8E-2 | 3.5E-1 | 6.9E-3 | 1.6E-1 | 4.6E-3 | 2.9E-2 |
| | TDM | 2.2E-2 | 6.8E-2 | 1.4E-2 | 1.2E-1 | 9.4E-3 | 8.3E-3 | 1.5E-2 | 3.8E-2 | 2.0E-2 | 7.7E-2 | 1.8E-2 | 3.7E-1 | 3.9E-3 | 1.8E-1 | 7.1E-3 | 2.1E-2 |
| | ReMasker | 4.9E-2 | 5.0E-2 | 2.2E-2 | 1.0E-1 | 1.4E-2 | 1.7E-2 | 7.5E-3 | 9.9E-3 | 1.4E-2 | 9.5E-2 | 3.8E-2 | 1.7E-1 | 2.5E-2 | 4.0E-1 | 1.0E-2 | 7.5E-3 |
| | **NewImp** | 2.5E-2 | 1.0E-1 | 3.9E-3 | 1.3E-1 | 1.9E-2 | 2.9E-2 | 8.4E-3 | 1.2E-2 | 9.0E-3 | 1.3E-1 | 8.5E-3 | 5.0E-2 | 7.1E-3 | 6.8E-1 | 5.8E-3 | 1.6E-2 |

Table F.4: Ablation study results at 30% missing rate.

| Scenario | NER | Joint | BT MAE | BT WASS | BCD MAE | BCD WASS | CC MAE | CC WASS | CBV MAE | CBV WASS | IS MAE | IS WASS | PK MAE | PK WASS | QB MAE | QB WASS | WQW MAE | WQW WASS |
|---|---|---|---|---|---|---|---|---|---|---|---|---|---|---|---|---|---|---|
| MAR | ✗ | ✗ | 0.96* | 3.82* | 1.05* | 20.2* | 1.04* | 5.47* | 0.86* | 5.81* | 0.67* | 20.2* | 1.06* | 15.6* | 0.72* | 22.5* | 0.79* | 6.49* |
| | ✗ | ✓ | _0.54_ | _0.42_ | _0.34_ | _0.82_ | _0.61*_ | _0.40*_ | _0.58*_ | _0.47*_ | _0.43*_ | _1.34_ | _0.46*_ | _1.25*_ | _0.47*_ | _3.56*_ | _0.55*_ | _0.64*_ |
| | ✓ | ✗ | 0.96* | 3.83* | 1.05* | 20.3* | 1.04* | 5.49* | 0.86* | 5.83* | 0.67* | 20.2* | 1.06* | 15.6* | 0.72* | 22.5* | 0.79* | 6.51* |
| | ✓ | ✓ | **0.52** | **0.38** | **0.34** | **0.82** | **0.35** | **0.25** | **0.31** | **0.20** | **0.39** | **1.31** | **0.44** | **1.21** | **0.45** | **3.50** | **0.46** | **0.55** |
| MCAR | ✗ | ✗ | 0.72* | 2.11* | 0.74* | 16.7* | 0.85* | 3.72* | 0.83* | 5.22* | 0.74* | 18.4* | 0.71* | 12.7* | 0.58* | 20.1* | 0.76* | 5.57* |
| | ✗ | ✓ | _0.52*_ | **0.17*** | **0.25** | **0.79** | _0.62*_ | _0.46*_ | _0.61*_ | _0.71*_ | 0.46 | **3.05** | _0.34_ | _1.09_ | _0.36*_ | _3.74*_ | _0.58*_ | _0.82*_ |
| | ✓ | ✗ | 0.72* | 2.12* | 0.73* | 16.8* | 0.86* | 3.73* | 0.83* | 5.24* | 0.74* | 18.4* | 0.71* | 12.7* | 0.58* | 20.1* | 0.76* | 5.60* |
| | ✓ | ✓ | **0.48** | _0.18_ | **0.25** | _0.80_ | **0.47** | **0.34** | **0.42** | **0.44** | **0.44** | 3.05 | **0.32** | **1.01** | **0.34** | **3.66** | **0.53** | **0.76** |
| MNAR | ✗ | ✗ | 0.81* | 2.47* | 0.89* | 18.2* | 0.87* | 3.85* | 0.85* | 5.26* | 0.69* | 17.6* | 0.87* | 13.0* | 0.64* | 20.6* | 0.77* | 5.71* |
| | ✗ | ✓ | _0.62_ | _0.37_ | **0.32** | _1.47_ | _0.61*_ | _0.47*_ | _0.64*_ | _0.79*_ | 0.44 | _2.79_ | _0.43*_ | _1.88*_ | _0.44*_ | _5.65_ | _0.60*_ | _0.87*_ |
| | ✓ | ✗ | 0.82* | 2.57* | 0.89* | 18.3* | 0.87* | 3.86* | 0.85* | 5.28* | 0.69* | 17.7* | 0.88* | 13.5* | 0.64* | 20.7* | 0.77* | 5.73* |
| | ✓ | ✓ | **0.60** | **0.35** | **0.32** | **1.46** | **0.44** | **0.34** | **0.46** | **0.52** | **0.40** | **2.68** | **0.39** | **1.56** | **0.42** | **5.57** | **0.55** | **0.81** |

*Kindly Note*: The best results are **bolded** and the second best results are underliend. "*" marks the results that NewImp significantly outperform with $p$-value$< 0.05$ over paired samples $t$-test.

Table F.5: Standard deviation of Ablation Study Results with missing rate at 30%.

| Scenario | NER | Joint | BT MAE | BT WASS | BCD MAE | BCD WASS | CC MAE | CC WASS | CBV MAE | CBV WASS | IS MAE | IS WASS | PK MAE | PK WASS | QB MAE | QB WASS | WQW MAE | WQW WASS |
|---|---|---|---|---|---|---|---|---|---|---|---|---|---|---|---|---|---|---|
| MAR | ✗ | ✗ | 6.1E-2 | 4.1E-1 | 4.8E-2 | 6.8E-1 | 1.1E-1 | 5.6E-1 | 4.6E-2 | 4.5E-1 | 4.4E-2 | 3.8E0 | 1.4E-1 | 3.9E0 | 7.2E-2 | 2.4E0 | 5.0E-2 | 4.1E-1 |
| | ✗ | ✓ | 6.9E-2 | 1.2E-1 | 2.7E-2 | 1.1E-1 | 2.8E-2 | 4.7E-2 | 2.5E-2 | 3.8E-2 | 2.7E-2 | 1.3E-1 | 1.0E-2 | 9.0E-2 | 1.8E-2 | 2.6E-1 | 2.8E-2 | 5.6E-2 |
| | ✓ | ✗ | 6.1E-2 | 4.1E-1 | 4.8E-2 | 6.8E-1 | 1.1E-1 | 5.6E-1 | 4.6E-2 | 4.5E-1 | 4.4E-2 | 3.8E0 | 1.4E-1 | 3.9E0 | 7.2E-2 | 2.4E0 | 5.0E-2 | 4.1E-1 |
| | ✓ | ✓ | 2.0E-2 | 4.0E-2 | 2.6E-2 | 1.1E-1 | 5.6E-2 | 6.4E-2 | 1.6E-2 | 2.2E-2 | 1.9E-2 | 1.1E-1 | 1.1E-2 | 8.8E-2 | 1.9E-2 | 2.7E-1 | 1.6E-2 | 3.3E-2 |
| MCAR | ✗ | ✗ | 9.9E-3 | 5.8E-2 | 1.0E-2 | 2.3E-1 | 3.6E-3 | 6.0E-2 | 3.2E-3 | 2.0E-2 | 9.5E-3 | 4.2E-1 | 1.5E-2 | 2.3E-1 | 1.1E-2 | 1.0E0 | 4.0E-3 | 1.4E-2 |
| | ✗ | ✓ | 5.0E-3 | 5.9E-3 | 2.8E-3 | 4.6E-2 | 1.2E-2 | 1.5E-2 | 9.4E-3 | 1.7E-2 | 6.3E-3 | 1.2E-1 | 1.0E-2 | 1.2E-1 | 7.3E-3 | 1.9E-1 | 9.3E-4 | 3.9E-3 |
| | ✓ | ✗ | 1.0E-2 | 5.7E-2 | 1.0E-2 | 2.3E-1 | 3.6E-3 | 6.0E-2 | 3.2E-3 | 2.1E-2 | 9.5E-3 | 4.2E-1 | 1.5E-2 | 2.3E-1 | 1.0E-2 | 1.0E0 | 4.0E-3 | 1.4E-2 |
| | ✓ | ✓ | 3.3E-3 | 3.7E-3 | 1.9E-3 | 4.6E-2 | 1.1E-2 | 1.8E-2 | 4.1E-3 | 1.8E-2 | 5.7E-3 | 1.1E-1 | 6.4E-3 | 3.7E-2 | 4.8E-3 | 1.7E-1 | 2.2E-3 | 1.1E-2 |
| MNAR | ✗ | ✗ | 4.2E-2 | 1.5E-1 | 2.3E-2 | 8.5E-1 | 3.2E-2 | 1.8E-1 | 1.2E-2 | 5.3E-2 | 6.9E-3 | 1.2E-1 | 3.2E-2 | 9.6E-1 | 1.7E-2 | 8.3E-1 | 1.4E-2 | 1.0E-1 |
| | ✗ | ✓ | 4.0E-2 | 1.4E-1 | 3.4E-3 | 1.3E-1 | 1.8E-2 | 2.1E-2 | 4.8E-3 | 1.7E-2 | 1.0E-2 | 1.3E-1 | 1.1E-2 | 1.7E-1 | 8.0E-3 | 7.0E-1 | 7.7E-3 | 1.3E-2 |
| | ✓ | ✗ | 4.8E-2 | 1.4E-1 | 2.4E-2 | 8.5E-1 | 3.3E-2 | 1.8E-1 | 1.2E-2 | 5.3E-2 | 6.9E-3 | 1.2E-1 | 1.9E-2 | 2.6E-1 | 1.6E-2 | 8.3E-1 | 1.4E-2 | 1.0E-1 |
| | ✓ | ✓ | 2.5E-2 | 1.0E-1 | 3.9E-3 | 1.3E-1 | 1.9E-2 | 2.9E-2 | 8.4E-3 | 1.2E-2 | 9.0E-3 | 1.3E-1 | 8.5E-3 | 5.0E-2 | 7.2E-3 | 6.8E-1 | 5.7E-3 | 1.7E-2 |

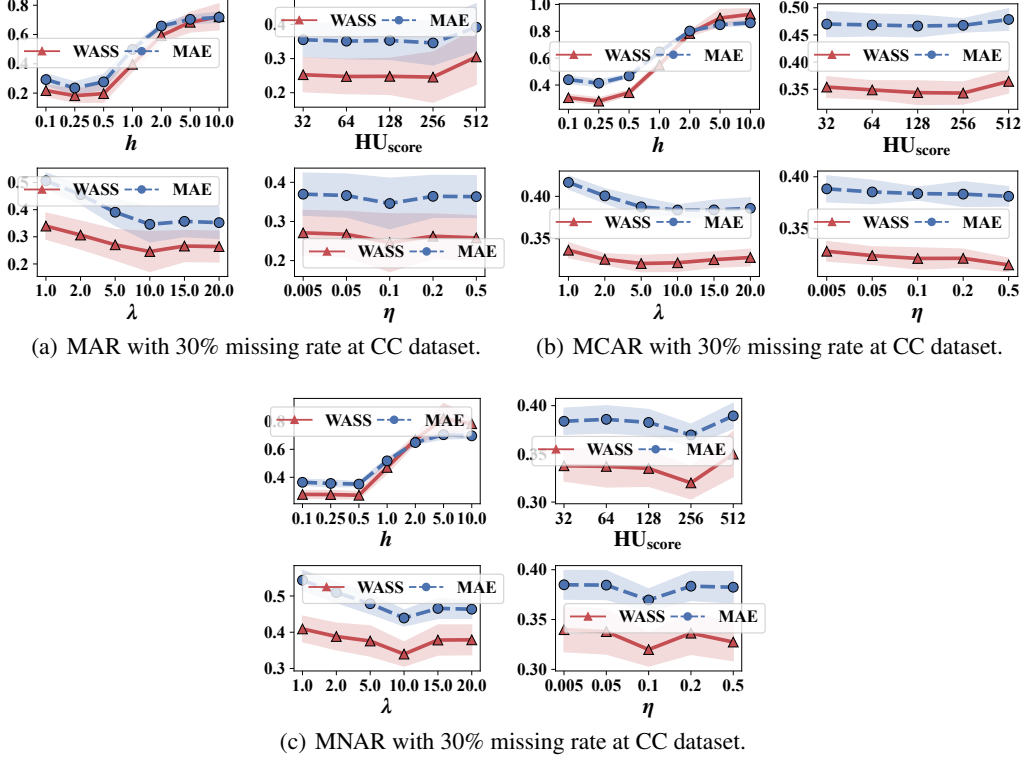

(a) MAR with 30% missing rate at CC dataset.    (b) MCAR with 30% missing rate at CC dataset.

(c) MNAR with 30% missing rate at CC dataset.

Figure F.2: Parameter sensitivity of NewImp on bandwidth for kernel function ($h$), hidden unit of score network $\text{HU}_{\text{score}}$, NER weight $\lambda$, and discretization step $\eta$ for Eq. (9) on CC dataset. Mean values and one standard deviation from mean are represented by scatters and shaded area, respectively.

### F.3  Empirical Evidence for Selecting RBF Function

In our derivation process, we specifically selected the RBF kernel to satisfy the 'zero boundary condition': $\lim_{\boldsymbol{X}^{(\text{joint})} \to \infty} K(\boldsymbol{X}^{(\text{joint})}, \tilde{\boldsymbol{X}}^{(\text{joint})}) = 0$ for the sake of avoiding the explicit density estimation of the intractable proposal distribution $r(\boldsymbol{X}^{(\text{joint})})$. This selection prompts an additional inquiry: What if we replaced the RBF kernel with another that does not fulfill the 'zero boundary condition'? To maintain the rigor of our analysis, we compared the performance of the NewImp with alternative kernel functions under identical settings. Consequently, we consider the following types of kernel functions:

- **linear kernel function (linear):** $K(\boldsymbol{X}^{(\text{joint})}, \tilde{\boldsymbol{X}}^{(\text{joint})}) = [\boldsymbol{X}^{(\text{joint})}][\tilde{\boldsymbol{X}}^{(\text{joint})}]^{\top}$

- **polynomial kernel function (poly):** $K(\boldsymbol{X}^{(\text{joint})}, \tilde{\boldsymbol{X}}^{(\text{joint})}) = \{[\boldsymbol{X}^{(\text{joint})}][\tilde{\boldsymbol{X}}^{(\text{joint})}]^{\top}\}^{2}$

- **sigmoid kernel function (sigmoid):** $K(\boldsymbol{X}^{(\text{joint})}, \tilde{\boldsymbol{X}}^{(\text{joint})}) = \tanh\{[\boldsymbol{X}^{(\text{joint})}][\tilde{\boldsymbol{X}}^{(\text{joint})}]^{\top}\}$

- **cosine similarity kernel function (cos):** $K(\boldsymbol{X}^{(\text{joint})}, \tilde{\boldsymbol{X}}^{(\text{joint})}) = \{\frac{[\boldsymbol{X}^{(\text{joint})}]}{\|[\boldsymbol{X}^{(\text{joint})}]\|}\}\{\frac{[\tilde{\boldsymbol{X}}^{(\text{joint})}]}{\|[\tilde{\boldsymbol{X}}^{(\text{joint})}]\|}\}^{\top}$

- **sine kernel function (sin):** $K(\boldsymbol{X}^{(\text{joint})}, \tilde{\boldsymbol{X}}^{(\text{joint})}) = \sin(|\boldsymbol{X}^{(\text{joint})} - \tilde{\boldsymbol{X}}^{(\text{joint})}|_{2}^{2})$

The experimental results are detailed in Table F.6 (For completeness, we also report the results under the MNAR scenario). From the results, it is evident that other kernel functions, which do not meet the 'zero boundary condition', perform significantly worse compared to the RBF kernel. This demonstrates the critical importance of selecting the appropriate kernel function for achieving accurate imputation results, thereby validating the choice of the RBF kernel for our NewImp approach.

Table F.6: Imputation accuracy vary different kernels at 30% missing rate.

| Scenario | Kernel | BT | | BCD | | CC | | CBV | | IS | | PK | | QB | | WQW | |
|---|---|---|---|---|---|---|---|---|---|---|---|---|---|---|---|---|---|
| | | MAE | WASS | MAE | WASS | MAE | WASS | MAE | WASS | MAE | WASS | MAE | WASS | MAE | WASS | MAE | WASS |
| MAR | linear | 0.77 | 0.73 | 0.83 * | 3.36 * | 0.85 * | 0.77 * | 0.83 * | 0.97 * | 0.72 * | 2.42 * | 0.80 * | 2.50 * | 0.70 * | 4.58 * | 0.76 * | 1.02 * |
| | poly | 1.12 * | 1.40 * | 1.44 * | 8.36 * | 1.06 * | 1.16 * | 1.08 * | 1.55 * | 1.07 * | 5.93 * | 1.33 * | 5.93 * | 1.28 * | 14 * | 1.02 * | 1.76 * |
| | sigmoid | 0.89 * | 1.02 * | 1.44 * | 11 * | 0.82 | 0.74 * | 0.81 * | 0.93 * | 1.17 * | 13 | 1.30 * | 9.87 * | 1.02 * | 7.19 * | 0.77 * | 1.04 * |
| | cos | 0.68 | 0.53 | 0.80 * | 3.14 * | 0.82 * | 0.73 * | 0.81 * | 0.92 * | 0.71 * | 2.37 * | 0.78 * | 2.37 * | 0.74 | 4.56 | 0.74 * | 0.98 * |
| | sin | 8.62 * | 95 | 8.74 * | 291 * | 15 * | 282 * | 18 * | 493 * | 11 * | 456 * | 10 * | 307 * | 8.00 * | 314 * | 15 * | 387 * |
| | **RBF** | **0.52** | **0.38** | **0.34** | **0.82** | **0.35** | **0.25** | **0.31** | **0.20** | **0.39** | **1.31** | **0.44** | **1.21** | **0.45** | **3.50** | **0.46** | **0.55** |
| MCAR | linear | 0.71 * | 0.45 * | 0.87 * | 5.87 * | 0.83 * | 0.84 * | 0.81 * | 1.28 * | 0.81 * | 5.65 * | 0.83 * | 4.14 * | 0.62 * | 6.31 * | 0.76 * | 1.25 * |
| | poly | 0.94 * | 0.88 * | 1.31 * | 12 * | 0.98 * | 1.21 * | 0.99 * | 1.85 * | 1.11 * | 9.94 * | 1.27 * | 8.64 * | 1.11 * | 19 * | 0.92 * | 1.95 * |
| | sigmoid | 0.74 * | 0.48 * | 0.97 * | 8.00 * | 0.81 * | 0.82 * | 0.80 * | 1.23 * | 0.92 * | 7.36 * | 0.94 * | 6.54 * | 0.76 * | 7.62 * | 0.77 * | 1.28 * |
| | cos | 0.70 * | 0.42 * | 0.84 * | 5.51 * | 0.81 * | 0.81 * | 0.80 * | 1.24 * | 0.80 * | 5.48 * | 0.81 * | 3.96 * | 0.63 * | 6.01 * | 0.74 * | 1.20 * |
| | sin | 9.76 * | 95 * | 7.77 * | 435 * | 14 * | 332 * | 12 * | 353 * | 8.27 * | 542 * | 8.62 * | 385 * | 7.36 * | 468 * | 11 * | 289 * |
| | **RBF** | **0.48** | **0.18** | **0.25** | **0.80** | **0.47** | **0.34** | **0.42** | **0.44** | **0.44** | **3.05** | **0.32** | **1.01** | **0.34** | **3.66** | **0.53** | **0.76** |
| MNAR | linear | 0.76 * | 0.60 | 0.90 * | 6.36 * | 0.82 * | 0.88 * | 0.83 * | 1.32 * | 0.79 * | 5.36 * | 0.86 * | 4.62 * | 0.65 * | 7.26 * | 0.76 * | 1.29 * |
| | poly | 0.92 * | 0.87 * | 1.26 * | 11 * | 0.98 * | 1.26 * | 1.01 * | 1.90 * | 1.09 * | 9.55 * | 1.19 * | 7.71 * | 1.14 * | 19 * | 0.92 * | 2.09 * |
| | sigmoid | 0.77 * | 0.59 | 1.01 * | 8.86 * | 0.80 * | 0.85 * | 0.82 * | 1.27 * | 0.91 * | 7.71 * | 0.99 * | 7.38 * | 0.83 * | 9.27 * | 0.77 * | 1.31 * |
| | cos | 0.73 * | 0.55 | 0.88 * | 6.19 * | 0.81 * | 0.84 * | 0.82 * | 1.27 * | 0.79 * | 5.26 * | 0.87 * | 4.77 * | 0.68 * | 7.17 * | 0.75 * | 1.24 * |
| | sin | 8.89 * | 84 * | 7.31 * | 362 * | 13 * | 300 * | 12 * | 338 * | 8.64 * | 571 * | 8.34 * | 362 * | 7.27 * | 469 * | 11 * | 300 * |
| | **RBF** | **0.60** | **0.35** | **0.32** | **1.46** | **0.44** | **0.34** | **0.46** | **0.52** | **0.40** | **2.68** | **0.39** | **1.56** | **0.42** | **5.57** | **0.55** | **0.81** |

*Kindly Note*: The best results are **bolded** and the second best results are underliend. "*" marks the results that RBF kernel significantly outperform with $p$-value$< 0.05$ over paired samples $t$-test.

Table F.7: Standard deviation of MAE and WASS metrics vary different kernels at 30% missing rate.

| Scenario | Kernel | BT | | BCD | | CC | | CBV | | IS | | PK | | QB | | WQW | |
|---|---|---|---|---|---|---|---|---|---|---|---|---|---|---|---|---|---|
| | | MAE | WASS | MAE | WASS | MAE | WASS | MAE | WASS | MAE | WASS | MAE | WASS | MAE | WASS | MAE | WASS |
| MAR | linear | 1.5E-1 | 2.4E-1 | 4.5E-2 | 4.3E-1 | 2.0E-2 | 3.0E-2 | 2.0E-2 | 5.0E-2 | 3.4E-2 | 1.7E-1 | 2.8E-2 | 3.2E-1 | 3.5E-2 | 2.6E-1 | 2.1E-2 | 3.7E-2 |
| | poly | 3.9E-2 | 7.8E-2 | 8.5E-3 | 2.4E-1 | 3.0E-2 | 7.8E-2 | 2.6E-2 | 4.6E-2 | 5.6E-2 | 5.0E-1 | 5.2E-2 | 6.1E-1 | 6.6E-2 | 5.7E-1 | 4.7E-2 | 6.6E-2 |
| | sigmoid | 4.1E-2 | 3.1E-1 | 4.1E-2 | 1.4E0 | 1.7E-2 | 6.7E-3 | 2.1E-2 | 5.4E-2 | 9.5E-2 | 9.9E0 | 1.2E-1 | 4.1E0 | 1.9E-2 | 1.7E-1 | 2.0E-2 | 6.2E-2 |
| | cos | 1.3E-1 | 2.6E-1 | 5.5E-2 | 5.1E-1 | 1.6E-2 | 1.1E-2 | 2.0E-2 | 4.8E-2 | 3.7E-2 | 1.9E-1 | 1.2E-2 | 2.3E-1 | 2.1E-2 | 8.7E-2 | 1.4E-2 | 4.1E-2 |
| | sin | 2.2E0 | 5.9E1 | 5.6E-1 | 4.7E1 | 2.8E0 | 1.0E2 | 2.5E0 | 1.4E2 | 2.5E-1 | 2.9E1 | 2.6E-1 | 1.1E1 | 2.0E-1 | 4.2E0 | 1.4E0 | 5.8E1 |
| | **RBF** | 2.0E-2 | 4.0E-2 | 2.6E-2 | 1.1E-1 | 5.6E-2 | 6.4E-2 | 1.6E-2 | 2.2E-1 | 1.9E-2 | 1.1E-1 | 1.1E-2 | 8.8E-2 | 1.9E-2 | 2.7E-1 | 1.6E-2 | 3.3E-2 |
| MCAR | linear | 3.1E-2 | 2.5E-2 | 8.1E-3 | 2.2E-1 | 7.9E-3 | 1.7E-2 | 4.6E-3 | 2.1E-2 | 1.2E-2 | 2.0E-1 | 3.2E-2 | 2.8E-1 | 8.2E-3 | 1.2E-1 | 6.4E-3 | 1.7E-2 |
| | poly | 3.2E-2 | 1.2E-1 | 9.0E-3 | 2.4E-1 | 8.2E-3 | 2.4E-2 | 8.5E-3 | 3.6E-2 | 2.2E-2 | 4.3E-1 | 1.6E-2 | 2.7E-1 | 1.1E-2 | 1.7E-1 | 9.2E-3 | 2.0E-2 |
| | sigmoid | 2.1E-2 | 3.3E-2 | 3.5E-3 | 1.6E-1 | 7.7E-3 | 1.1E-2 | 5.3E-3 | 2.3E-2 | 4.1E-3 | 1.7E-1 | 1.2E-2 | 7.0E-1 | 7.7E-3 | 1.5E-1 | 6.7E-3 | 1.8E-2 |
| | cos | 2.8E-2 | 4.5E-2 | 1.0E-2 | 2.5E-1 | 8.4E-3 | 1.7E-2 | 4.8E-3 | 2.1E-2 | 1.4E-2 | 2.0E-1 | 2.1E-2 | 2.1E-1 | 1.1E-2 | 1.3E-1 | 6.9E-3 | 1.9E-2 |
| | sin | 1.3E0 | 2.2E1 | 3.4E-1 | 3.1E1 | 1.1E0 | 5.7E1 | 1.0E0 | 7.6E1 | 1.4E-1 | 1.3E1 | 4.2E-1 | 4.4E1 | 2.6E-1 | 2.2E1 | 1.3E0 | 9.4E1 |
| | **RBF** | 3.3E-3 | 3.7E-3 | 1.9E-3 | 4.6E-2 | 1.1E-2 | 1.8E-2 | 4.1E-3 | 1.8E-2 | 5.7E-3 | 1.1E-1 | 6.4E-3 | 3.7E-2 | 4.7E-3 | 1.7E-1 | 2.2E-3 | 1.1E-2 |
| MNAR | linear | 3.0E-2 | 1.1E-1 | 2.5E-2 | 3.0E-1 | 2.3E-2 | 5.1E-2 | 2.1E-3 | 1.9E-2 | 2.3E-2 | 3.0E-1 | 4.4E-2 | 3.5E-1 | 1.2E-2 | 4.7E-1 | 4.2E-3 | 1.3E-2 |
| | poly | 1.6E-2 | 9.5E-2 | 1.4E-2 | 3.2E-1 | 2.6E-2 | 5.4E-2 | 1.2E-2 | 6.2E-2 | 1.6E-2 | 2.0E-1 | 1.9E-2 | 1.5E-1 | 6.8E-3 | 5.3E-1 | 5.1E-3 | 8.4E-2 |
| | sigmoid | 3.4E-2 | 1.4E-1 | 1.8E-2 | 2.2E-1 | 3.2E-2 | 6.8E-2 | 8.0E-4 | 1.6E-2 | 2.5E-2 | 6.8E-1 | 3.0E-3 | 1.7E-1 | 1.4E-2 | 4.5E-1 | 5.7E-3 | 1.1E-2 |
| | cos | 4.3E-2 | 1.3E-1 | 2.9E-2 | 3.4E-1 | 2.6E-2 | 5.8E-2 | 1.8E-3 | 1.6E-2 | 2.5E-2 | 3.4E-1 | 4.2E-2 | 3.6E-1 | 8.0E-3 | 5.4E-1 | 4.6E-3 | 1.1E-2 |
| | sin | 1.0E0 | 2.2E1 | 2.3E-1 | 1.8E1 | 1.4E0 | 8.7E1 | 5.5E-1 | 5.0E1 | 3.6E-1 | 1.8E1 | 1.7E-1 | 1.8E1 | 2.8E-1 | 3.3E1 | 1.2E0 | 7.1E1 |
| | **RBF** | 2.5E-2 | 1.0E-1 | 3.9E-3 | 1.3E-1 | 1.9E-2 | 2.9E-2 | 8.4E-3 | 1.2E-2 | 9.0E-3 | 1.3E-1 | 8.5E-3 | 5.0E-2 | 7.2E-3 | 6.8E-1 | 5.7E-3 | 1.7E-2 |

## F.4 Time Complexity Analysis

In this section, we present an analysis of the complexity of time for our NewImp approach. The complexity analysis is based on the algorithms described in Algorithm 1. We begin by estimating the time complexity of the score function $\nabla_{\boldsymbol{X}^{(\text{joint})}} \log \hat{p}(\boldsymbol{X}^{(\text{joint})})$. Assuming the number of layers in the neural network that parameterize $\nabla_{\boldsymbol{X}^{(\text{joint})}} \log \hat{p}(\boldsymbol{X}^{(\text{joint})})$ is L and each layer has an equal number of hidden units denoted as $\text{HU}_{\text{score}}$, the time complexity for the imputation algorithm defined in Algorithm 1 is detailed as follows:

1. DSM **Training Part (step 5):** Building on the previous item, the time complexity for the DSM training algorithm defined in Algorithm 3 is given as:

$$\mathcal{O}\left[4 \times \text{N} \times \left(\text{D} \times \text{HU}_{\text{score}} + (\text{L}-1) \times \text{HU}_{\text{score}}^2\right)\right], \tag{F.1}$$

where the factor of 4 comprises three distinct components: backward propagation (1), forward propagation (1), and the acquisition of the sample-wise score function (2). Note that the network parameter size is substantially smaller than the number of data points, thereby making the forward computation of the score function the primary factor in time complexity.

2. **Imputation Part (step 7):**

   • **Score function computation:** The time complexity for computing the score function is expressed as:

$$\mathcal{O}\left[2 \times \text{N} \times \left(\text{D} \times \text{HU}_{\text{score}} + (\text{L}-1) \times \text{HU}_{\text{score}}^2\right)\right], \tag{F.2}$$

   where the factor 2 accounts for the backward propagation needed during the score function computation.

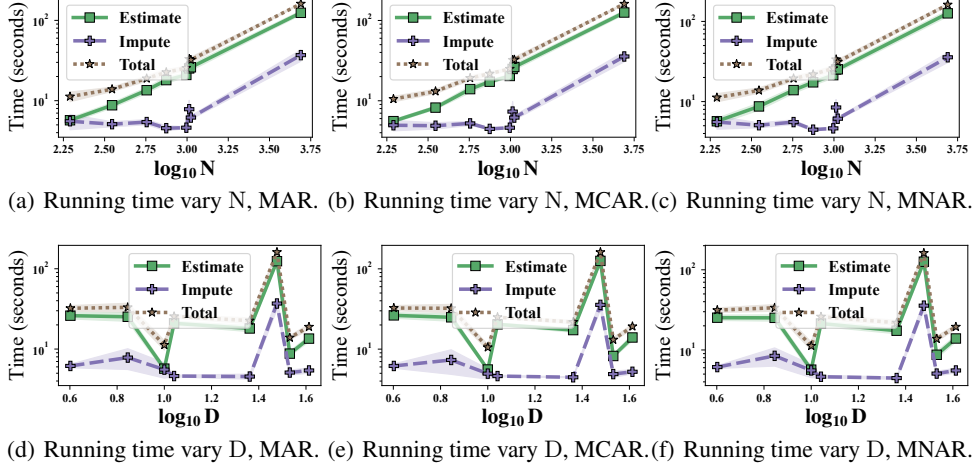

(a) Running time vary N, MAR. (b) Running time vary N, MCAR.(c) Running time vary N, MNAR.

(d) Running time vary D, MAR. (e) Running time vary D, MCAR.(f) Running time vary D, MNAR.

Figure F.3: Average computation time, where 'Estimate' indicates the 'DSM Training Algorithm' (step 5 of Algorithm 1), and 'Impute' indicates the imputation algorithm (step 7 of Algorithm 1). The scatters and shaded areas indicate the mean and one standard deviation from the mean, respectively.

- **Kernel function and its gradient:** Employing the RBF kernel $K(\boldsymbol{X}, \tilde{\boldsymbol{X}}) := \exp\left(-\frac{\|\boldsymbol{X}-\tilde{\boldsymbol{X}}\|^2}{2h^2}\right)$, the gradient with respect to $\tilde{\boldsymbol{X}}$ is analytically determined as:

$$[\nabla_{\tilde{\boldsymbol{X}}} K(\boldsymbol{X}, \tilde{\boldsymbol{X}})][:,j] = -\frac{1}{h^2}\left\{[K(\boldsymbol{X}, \tilde{\boldsymbol{X}}) \times \tilde{\boldsymbol{X}}][:,j] + \tilde{\boldsymbol{X}}[:,j] \odot \sum_{j=1}^{D} K(\boldsymbol{X}, \tilde{\boldsymbol{X}})[:,j]\right\}. \tag{F.3}$$

The time complexities for calculating the kernel function and its gradient are specified in Eqs. (F.4) and (F.5):

$$\mathcal{O}\left[N^2 \times D + N^2\right], \tag{F.4}$$

$$\mathcal{O}\left[N^2 \times D + N^2 + N \times D\right]. \tag{F.5}$$

Based on the abovementioned analysis, we explore how computational complexity varies with different dataset sizes N and the number of features D, as shown in Figs. F.3 (a) and (b), respectively. From these figures, it is evident that computational time increases with the dataset size N. However, changes in the number of features D do not significantly affect the computation time. This observation underscores that the primary determinant of computational complexity in our context is the dataset size, aligning with our theoretical analysis, which indicates a quadratic relationship between time complexity and the size of the dataset N for the 'Imputation' part, and $N \gg D$ for the 'DSM Training' part, aligning with our theoretical analysis.

Moreover, the data reveals that the total computational time is predominantly governed by 'Estimation' part of our NewImp approach. This suggests that the training of the score function represents a critical bottleneck in the efficiency of the NewImp algorithm. Therefore, accelerating the NewImp algorithm crucially hinges on reducing the computational demands of the 'Estimation' part.

## F.5 Convergence Analysis

In this section, we want to discuss the convergence of the proposed NewImp approach, prior to delving into this discussion, it is essential to establish a clear definition of convergence:

**Definition F.1.** *A sequence* $\{\mathcal{F}_1, \mathcal{F}_2, ..., \mathcal{F}_T\}$ *is said to be convergent if there exists a real number* $\mathcal{G}$ *such that for any given positive number* $\varepsilon$ *($\varepsilon > 0$), there exists a positive integer* $N$, *such that for all indices* $n$ *greater than* $N$, *the corresponding terms* $\mathcal{F}_n, n \geq N$ *satisfy the inequality* $|\mathcal{F}_n - \mathcal{G}| < \varepsilon$.

Based on Definition F.1, if a sequence is either monotonically increasing or monotonically decreasing and bounded (either bounded above or bounded below), then it is guaranteed to converge according

to the celebrated monotone convergence theorem (Section 3.14 in reference [45]). Based on this, the convergence of the 'Imputation' part (step 7 of Algorithm 1) and DSM training part (step 5 of Algorithm 1) are proposed in the proceeding parts.

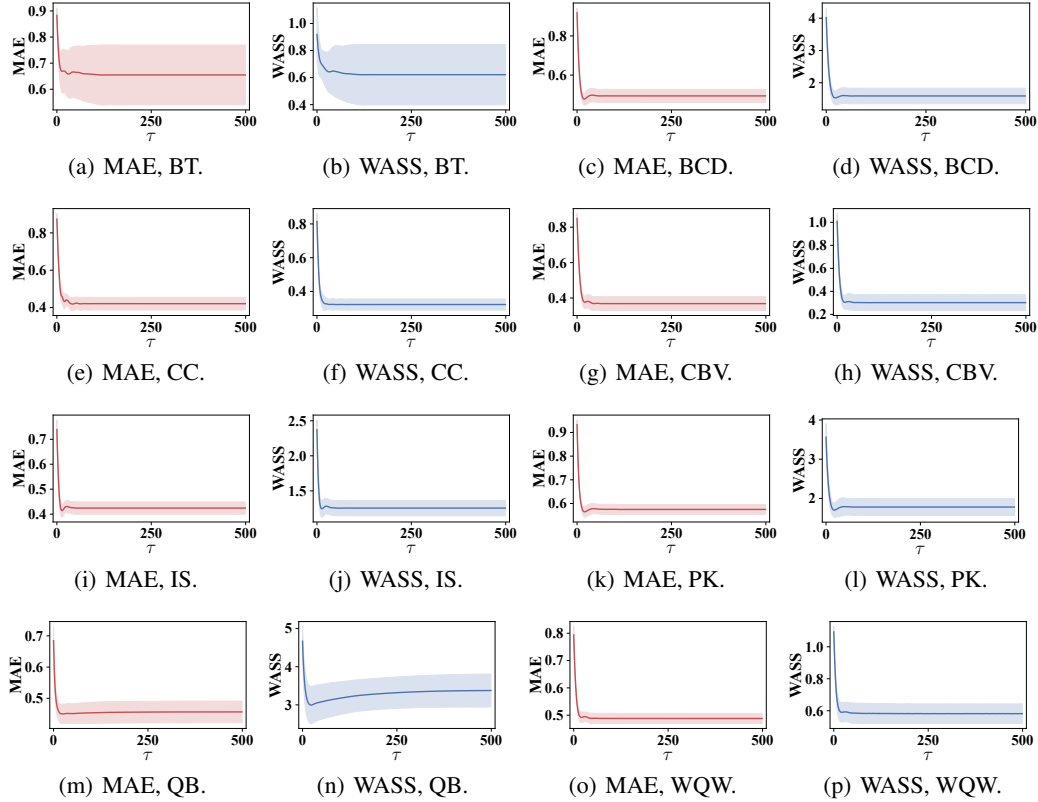

Figure F.4: Evolution of evaluation metrics along iteration time $\tau$ under MAR scenario at 30% missing rate. The shaded area indicates the $\pm$ 1.0 standard deviation uncertainty interval.

### F.5.1 Convergence Analysis of the Imputation Part

In this section, we explore the convergence of the imputation part as defined in step 7 of Algorithm 1 within our NewImp approach. Based on this, we first prove the following proposition for the convergence in the 'Imputation' part:

**Proposition F.1.** *The convergence of the imputation part can be guaranteed, given that the discretization step size $\eta$ is small enough.*

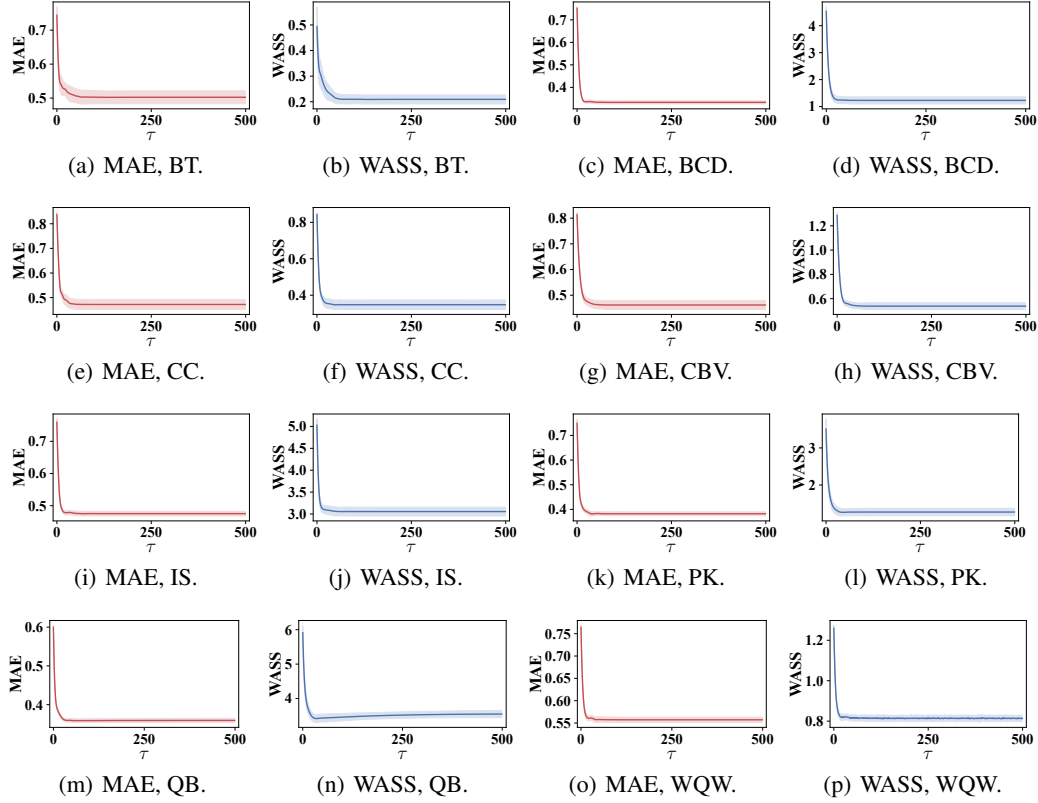

Figure F.5: Evolution of evaluation metrics along iteration time $\tau$ under MCAR scenario at 30% missing rate. The shaded area indicates the $\pm 1.0$ standard deviation uncertainty interval.

*Proof.* First, let us reformulate the velocity field as follows:

$$
\begin{aligned}
&u(\boldsymbol{X}^{(\text{joint})}) \\
&= \mathbb{E}_{r(\tilde{\boldsymbol{X}}^{(\text{joint})})} \left\{
\begin{array}{l}
-\lambda \nabla_{\tilde{\boldsymbol{X}}^{(\text{miss})}} K(\boldsymbol{X}^{(\text{joint})}, \tilde{\boldsymbol{X}}^{(\text{joint})}) \\
\quad + [\nabla_{\tilde{\boldsymbol{X}}^{(\text{miss})}} \log \hat{p}(\tilde{\boldsymbol{X}}^{(\text{joint})})]^{\top} K(\boldsymbol{X}^{(\text{joint})}, \tilde{\boldsymbol{X}}^{(\text{joint})})
\end{array}
\right\} \\
&\overset{(\text{i})}{=} \mathbb{E}_{r(\tilde{\boldsymbol{X}}^{(\text{joint})})} \left\{
\begin{array}{l}
\lambda [\nabla_{\tilde{\boldsymbol{X}}^{(\text{miss})}} \log r(\tilde{\boldsymbol{X}}^{(\text{joint})})]^{\top} K(\boldsymbol{X}^{(\text{joint})}, \tilde{\boldsymbol{X}}^{(\text{joint})}) \\
\quad + [\nabla_{\tilde{\boldsymbol{X}}^{(\text{miss})}} \log \hat{p}(\tilde{\boldsymbol{X}}^{(\text{joint})})]^{\top} K(\boldsymbol{X}^{(\text{joint})}, \tilde{\boldsymbol{X}}^{(\text{joint})})
\end{array}
\right\} \\
&= \int r(\tilde{\boldsymbol{X}}^{(\text{joint})}) \left\{
\begin{array}{l}
\lambda \nabla_{\tilde{\boldsymbol{X}}^{(\text{miss})}} \log r(\tilde{\boldsymbol{X}}^{(\text{joint})}) \\
\quad + \nabla_{\tilde{\boldsymbol{X}}^{(\text{miss})}} \log \hat{p}(\tilde{\boldsymbol{X}}^{(\text{joint})})
\end{array}
\right\}^{\top} K(\boldsymbol{X}^{(\text{joint})}, \tilde{\boldsymbol{X}}^{(\text{joint})}) \mathrm{d}\tilde{\boldsymbol{X}}^{(\text{joint})} \\
&= \int \left\{
\begin{array}{l}
\lambda \nabla_{\tilde{\boldsymbol{X}}^{(\text{miss})}} \log r(\tilde{\boldsymbol{X}}^{(\text{joint})}) \\
\quad + \nabla_{\tilde{\boldsymbol{X}}^{(\text{miss})}} \log \hat{p}(\tilde{\boldsymbol{X}}^{(\text{joint})})
\end{array}
\right\}^{\top} K(\boldsymbol{X}^{(\text{joint})}, \tilde{\boldsymbol{X}}^{(\text{joint})}) \mathrm{d}r(\tilde{\boldsymbol{X}}^{(\text{joint})}),
\end{aligned}
\tag{F.6}
$$

where (i) is based on integration by parts.

Based on this reformulation, the inner product can be given as follows:

$$
\begin{aligned}
&\frac{\mathrm{d}\mathcal{F}_{\text{joint-NER}}}{\mathrm{d}\tau} \\
&= \int \left\langle \nabla_{\boldsymbol{X}^{(\text{miss})}} \frac{\delta \mathcal{F}_{\text{joint-NER}}}{\delta r(\boldsymbol{X}^{(\text{joint})})}, u(\boldsymbol{X}^{(\text{joint})}) \right\rangle \mathrm{d}r(\boldsymbol{X}^{(\text{miss})}) \\
&= \iint \left\{ \begin{aligned} &\lambda \nabla_{\tilde{\boldsymbol{X}}^{(\text{miss})}} \log r(\tilde{\boldsymbol{X}}^{(\text{joint})}) \\ &+ \nabla_{\tilde{\boldsymbol{X}}^{(\text{miss})}} \log \hat{p}(\tilde{\boldsymbol{X}}^{(\text{joint})}) \end{aligned} \right\}^{\top} K(\boldsymbol{X}^{(\text{joint})}, \tilde{\boldsymbol{X}}^{(\text{joint})}) \times \\
&\qquad\qquad\qquad\qquad \left\{ \begin{aligned} &\lambda \nabla_{\boldsymbol{X}^{(\text{miss})}} \log r(\boldsymbol{X}^{(\text{joint})}) \\ &+ \nabla_{\boldsymbol{X}^{(\text{miss})}} \log \hat{p}(\boldsymbol{X}^{(\text{joint})}) \end{aligned} \right\} \mathrm{d}r(\tilde{\boldsymbol{X}}^{(\text{joint})}) \mathrm{d}r(\boldsymbol{X}^{(\text{joint})}) \\
&\overset{\text{(i)}}{\geq} 0,
\end{aligned}
\tag{F.7}
$$

where the (i) is predicated on the requirement that the kernel function, $K(\cdot, \cdot)$, is semi-positive definite. Consequently, according to the abovementioned derivation, we can conclude that the evolution of $\mathcal{F}_{\text{joint-NER}}$ is monotonic increasing along $\tau$. Furthermore, $\mathcal{F}_{\text{joint-NER}}$ satisfies the following inequality:

$$
\begin{aligned}
&\mathcal{F}_{\text{joint-NER}} \\
&\leq \mathcal{F}_{\text{joint-NER}} - (\lambda + 1)\mathbb{E}_{r(\boldsymbol{X}^{(\text{joint})})}[\log r(\boldsymbol{X}^{(\text{joint})})] \\
&= -\mathbb{D}_{\text{KL}}\left[ r(\boldsymbol{X}^{(\text{joint})}) \| \hat{p}(\boldsymbol{X}^{(\text{joint})}) \right] \\
&\leq 0,
\end{aligned}
\tag{F.8}
$$

which indicates that $\mathcal{F}_{\text{joint-NER}}$ is upper-bounded by 0.

According to Eqs. (F.7) and (F.8), the cost functional $\mathcal{F}_{\text{joint-NER}}$, driven by the velocity field $u(\boldsymbol{X}^{(\text{joint})})$ along $\tau$, converges. Building on this, employing a smaller step size $\eta$ results in the iteration curve of $\mathcal{F}_{\text{joint-NER}}$ more closely approximating the ODE defined in Eq. (F.7). Consequently, a smaller $\eta$ leads to a sequence where $\mathcal{F}_{\text{joint-NER}}$ monotonically increases, aligning with the theoretical expectations of the ODE behavior. □

Unfortunately, directly obtaining $\mathcal{F}_{\text{joint-NER}}$ is intractable. Nevertheless, we can still observe the changes in WASS and MAE across iteration time $\tau$ to demonstrate the convergence of the 'Impute' part. To this end, we present the convergence trends along $\tau$ in Figs. F.4 to F.6. These figures illustrate that both MAE and WASS generally decrease as the iteration epochs increase and eventually stabilize after $\tau = 250$. This observed behavior supports our theoretical findings regarding the convergence of the 'Imputation' part.

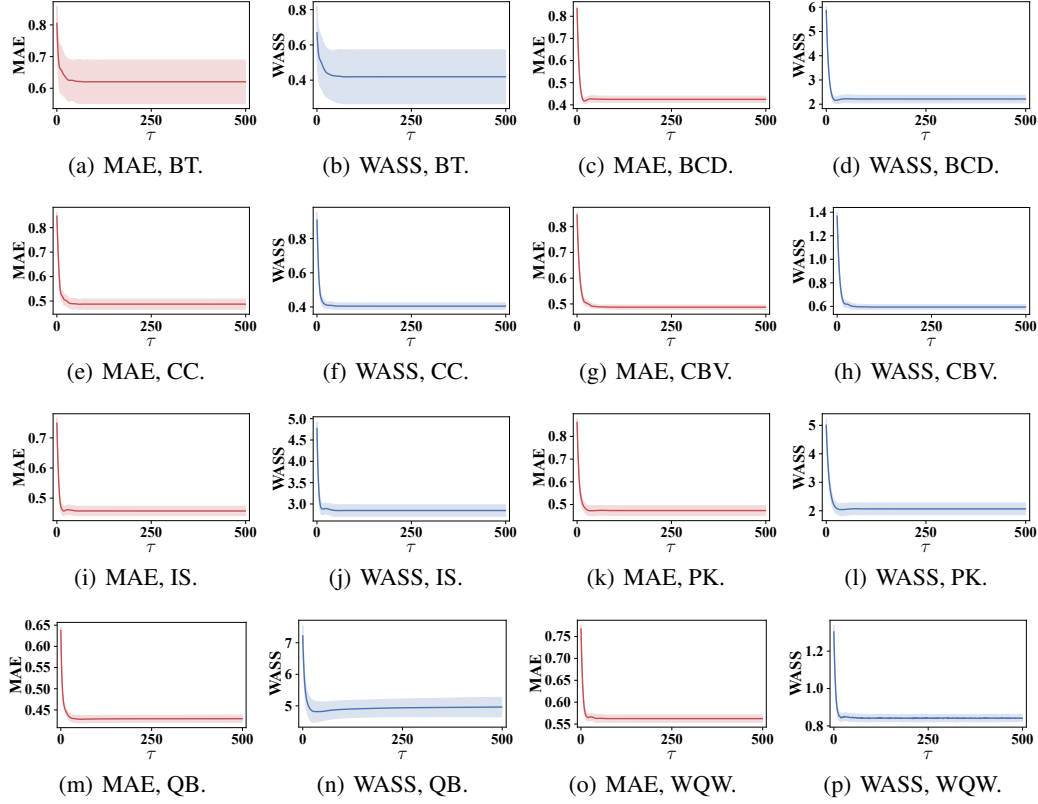

Figure F.6: Evolution of evaluation metrics along iteration time $\tau$ under MNAR scenario at 30% missing rate. The shaded area indicates the $\pm$ 1.0 standard deviation uncertainty interval.

### F.5.2 Convergence Analysis of the DSM Training

Similarly, we can also give the proposition of the DSM training algorithm located in step 5 of Algorithm 1, and summarized in Algorithm 3:

**Proposition F.2.** *The convergence of the* DSM *training algorithm can be guaranteed, given that the learning rate* $lr$ *is small enough.*

*Proof.* In the beginning, let us reformulate the parameter learning procedure of the DSM training algorithm as follows:

$$\theta_{\tau+1} = \theta_\tau - lr \times \nabla_\theta \mathcal{L}_{\text{DSM}}|_{\theta=\theta_\tau}, \tag{F.9}$$

which can be further reformulated as follows:

$$\frac{\theta_{\tau+1} - \theta_\tau}{lr} = -\nabla_\theta \mathcal{L}_{\text{DSM}}|_{\theta=\theta_\tau}$$

$$\Rightarrow \lim_{lr \to 0} \frac{\theta_{\tau+1} - \theta_\tau}{lr} = -\nabla_\theta \mathcal{L}_{\text{DSM}}|_{\theta=\theta_\tau} \tag{F.10}$$

$$\Rightarrow \frac{d\theta}{d\tau} = -\nabla_\theta \mathcal{L}_{\text{DSM}}$$

Meanwhile, note that:

$$\frac{d\mathcal{L}_{\text{DSM}}}{d\tau} = \left\langle \nabla_\theta \mathcal{L}_{\text{DSM}}, \frac{d\theta}{d\tau} \right\rangle. \tag{F.11}$$

Plugging Eq. (F.10) into Eq. (F.11), we can get the following result:

$$\frac{d\mathcal{L}_{\text{DSM}}}{d\tau} = -\left\langle \nabla_\theta \mathcal{L}_{\text{DSM}}, \nabla_\theta \mathcal{L}_{\text{DSM}} \right\rangle \leq 0, \tag{F.12}$$

which indicates that the iterative procedure for $\mathcal{L}_{\text{DSM}}$ is monotonic decreasing along $\tau$.

Finally, recall Eq. (15), we can know that the following condition holds:

$$\mathcal{L}_{\text{DSM}} \geq 0. \tag{F.13}$$

Building on this, employing a smaller step size $lr$ results in the iteration curve of $\mathcal{L}_{\text{DSM}}$ more closely approximating the ODE defined in Eq. (F.11). Consequently, a smaller $lr$ leads to a sequence where $\mathcal{L}_{\text{DSM}}$ monotonically decreases, aligning with the theoretical expectations of the ODE behavior.

□

Based on this proposition, we plot the evolution of $\mathcal{L}_{\text{DSM}}$ along time $\tau$ in Fig. F.7. These figures illustrate that the $\mathcal{L}_{\text{DSM}}$ generally decreases as the iteration epochs increase. This observed behavior supports our theoretical findings regarding the convergence of the DSM training algorithm.

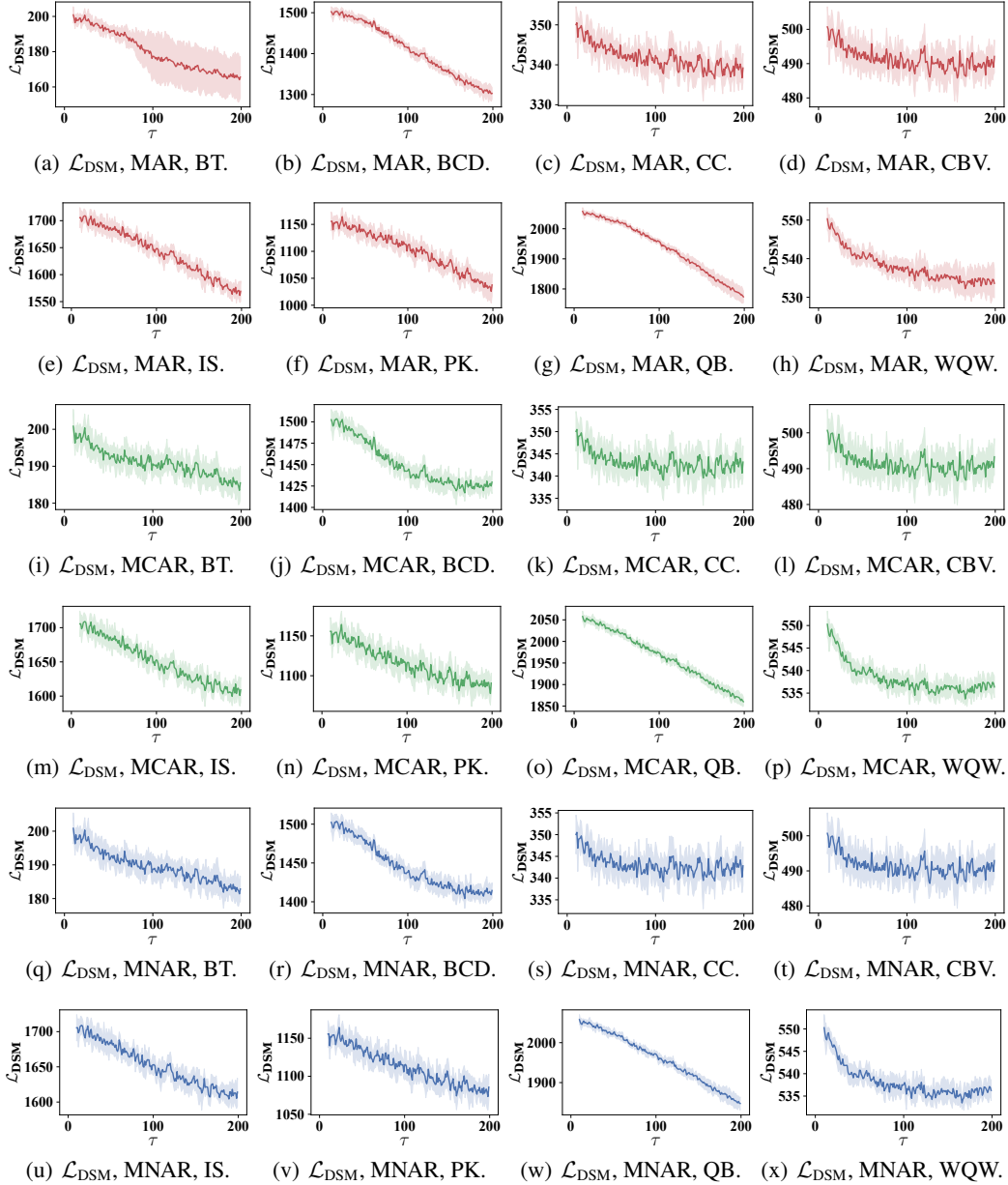

(a) $\mathcal{L}_{\text{DSM}}$, MAR, BT.　(b) $\mathcal{L}_{\text{DSM}}$, MAR, BCD.　(c) $\mathcal{L}_{\text{DSM}}$, MAR, CC.　(d) $\mathcal{L}_{\text{DSM}}$, MAR, CBV.

(e) $\mathcal{L}_{\text{DSM}}$, MAR, IS.　(f) $\mathcal{L}_{\text{DSM}}$, MAR, PK.　(g) $\mathcal{L}_{\text{DSM}}$, MAR, QB.　(h) $\mathcal{L}_{\text{DSM}}$, MAR, WQW.

(i) $\mathcal{L}_{\text{DSM}}$, MCAR, BT.　(j) $\mathcal{L}_{\text{DSM}}$, MCAR, BCD.　(k) $\mathcal{L}_{\text{DSM}}$, MCAR, CC.　(l) $\mathcal{L}_{\text{DSM}}$, MCAR, CBV.

(m) $\mathcal{L}_{\text{DSM}}$, MCAR, IS.　(n) $\mathcal{L}_{\text{DSM}}$, MCAR, PK.　(o) $\mathcal{L}_{\text{DSM}}$, MCAR, QB.　(p) $\mathcal{L}_{\text{DSM}}$, MCAR, WQW.

(q) $\mathcal{L}_{\text{DSM}}$, MNAR, BT.　(r) $\mathcal{L}_{\text{DSM}}$, MNAR, BCD.　(s) $\mathcal{L}_{\text{DSM}}$, MNAR, CC.　(t) $\mathcal{L}_{\text{DSM}}$, MNAR, CBV.

(u) $\mathcal{L}_{\text{DSM}}$, MNAR, IS.　(v) $\mathcal{L}_{\text{DSM}}$, MNAR, PK.　(w) $\mathcal{L}_{\text{DSM}}$, MNAR, QB.　(x) $\mathcal{L}_{\text{DSM}}$, MNAR, WQW.

Figure F.7: Evolution of $\mathcal{L}_{\text{DSM}}$, the loss function of 'Estimation' part along iteration time $\tau$ at 30% missing rate. The shaded area indicates the $\pm 1.0$ standard deviation uncertainty interval. The results of $\mathcal{L}_{\text{DSM}}$ are smoothed by exponential moving average with $\alpha = 0.60$.

## F.6 Downstream Task Comparison

To further substantiate the rigor of our manuscript and demonstrate the efficacy of the proposed NewImp approach, we conduct downstream task comparisons as detailed in this subsection. Initially, we evaluate the classification performance on imputed data using the following protocol: 1) Selection of datasets with non-binary labels. 2) Post-imputation, we train a support vector machine equipped with an RBF kernel and an automatic kernel coefficient. We assess the model's performance using 5-fold cross-validation, reporting both the mean and standard deviation of the accuracies across 10 runs with different random seeds. In this procedure, we select classification accuracy as our evaluation metric. 3) Additionally, we include the accuracy of ground-truth data for reference. The comparative results are presented in Table F.8. From Table F.8, it can be seen that the NewImp approach generally has the best performance among all baseline models, this phenomenon reflects the superiority of the proposed NewImp approach in a further way.

Table F.8: Classification accuracy results at 30% missing rate.

| Scenario | Model | BCD | CBV | IS | QB | WQW |
|---|---|---|---|---|---|---|
| MAR | CSDI_T | $0.677^*_{\pm1.39\text{E-}17}$ | $0.069^*_{\pm0.00\text{E}0}$ | $0.603^*_{\pm1.39\text{E-}17}$ | $0.640^*_{\pm0.00\text{E}0}$ | $0.441^*_{\pm1.39\text{E-}17}$ |
| | MissDiff | $0.839^*_{\pm4.93\text{E-}17}$ | $0.406^*_{\pm2.47\text{E-}17}$ | $0.900^*_{\pm3.70\text{E-}17}$ | $0.783^*_{\pm4.93\text{E-}17}$ | $0.490^*_{\pm1.85\text{E-}17}$ |
| | GAIN | $0.971^*_{\pm1.23\text{E-}17}$ | $0.519^*_{\pm1.85\text{E-}17}$ | $\underline{0.941}^*_{\pm4.93\text{E-}17}$ | $0.794^*_{\pm3.70\text{E-}17}$ | $\underline{0.512}^*_{\pm6.17\text{E-}17}$ |
| | MIRACLE | $0.966^*_{\pm4.93\text{E-}17}$ | $0.579^*_{\pm3.70\text{E-}17}$ | $0.827^*_{\pm2.47\text{E-}17}$ | $0.798^*_{\pm2.47\text{E-}17}$ | $0.488^*_{\pm3.08\text{E-}17}$ |
| | MIWAE | $0.968^*_{\pm6.94\text{E-}17}$ | $0.567^*_{\pm2.08\text{E-}17}$ | $0.939^*_{\pm5.55\text{E-}17}$ | $0.858^*_{\pm8.33\text{E-}17}$ | $0.482^*_{\pm2.78\text{E-}17}$ |
| | Sink | $0.958^*_{\pm0.00\text{E}0}$ | $0.499^*_{\pm1.39\text{E-}17}$ | $0.891^*_{\pm4.16\text{E-}17}$ | $0.798^*_{\pm6.94\text{E-}17}$ | $0.477^*_{\pm1.39\text{E-}17}$ |
| | TDM | $0.969^*_{\pm4.16\text{E-}17}$ | $\underline{0.581}^*_{\pm5.55\text{E-}17}$ | $0.938^*_{\pm4.16\text{E-}17}$ | $0.796^*_{\pm5.55\text{E-}17}$ | $0.505^*_{\pm4.86\text{E-}17}$ |
| | ReMasker | $\underline{0.973}^*_{\pm2.78\text{E-}17}$ | $0.517^*_{\pm2.08\text{E-}17}$ | $0.935^*_{\pm0.00\text{E}0}$ | $\underline{0.859}_{\pm4.16\text{E-}17}$ | $0.489^*_{\pm1.85\text{E-}17}$ |
| | **NewImp** | $\mathbf{0.974}_{\pm6.17\text{E-}17}$ | $\mathbf{0.595}_{\pm6.17\text{E-}17}$ | $\mathbf{0.947}_{\pm4.93\text{E-}17}$ | $\mathbf{0.860}_{\pm3.70\text{E-}17}$ | $\mathbf{0.513}_{\pm4.86\text{E-}17}$ |
| MCAR | CSDI_T | $0.593^*_{\pm1.39\text{E-}17}$ | $0.066^*_{\pm2.60\text{E-}18}$ | $0.609^*_{\pm1.39\text{E-}17}$ | $0.656^*_{\pm0.00\text{E}0}$ | $0.441^*_{\pm1.39\text{E-}17}$ |
| | MissDiff | $0.756^*_{\pm3.70\text{E-}17}$ | $0.327^*_{\pm3.70\text{E-}17}$ | $0.883^*_{\pm6.17\text{E-}17}$ | $0.732^*_{\pm7.40\text{E-}17}$ | $0.478^*_{\pm3.08\text{E-}17}$ |
| | GAIN | $0.964^*_{\pm7.40\text{E-}17}$ | $\underline{0.489}^*_{\pm3.08\text{E-}17}$ | $\underline{0.929}^*_{\pm2.47\text{E-}17}$ | $\underline{0.837}^*_{\pm6.17\text{E-}17}$ | $0.491^*_{\pm3.70\text{E-}17}$ |
| | MIRACLE | $0.923^*_{\pm6.17\text{E-}17}$ | $0.450^*_{\pm2.47\text{E-}17}$ | $0.709^*_{\pm7.40\text{E-}17}$ | $0.770^*_{\pm0.00\text{E}0}$ | $0.474^*_{\pm3.08\text{E-}17}$ |
| | MIWAE | $0.957^*_{\pm6.94\text{E-}17}$ | $0.451^*_{\pm1.39\text{E-}17}$ | $0.917^*_{\pm5.55\text{E-}17}$ | $0.831^*_{\pm4.16\text{E-}17}$ | $\underline{0.492}_{\pm2.08\text{E-}17}$ |
| | Sink | $0.950^*_{\pm0.00\text{E}0}$ | $0.446^*_{\pm2.08\text{E-}17}$ | $0.877^*_{\pm6.94\text{E-}17}$ | $0.768^*_{\pm0.00\text{E}0}$ | $0.462^*_{\pm2.08\text{E-}17}$ |
| | TDM | $0.961^*_{\pm8.33\text{E-}17}$ | $0.486^*_{\pm3.47\text{E-}17}$ | $0.922^*_{\pm5.55\text{E-}17}$ | $0.836^*_{\pm5.55\text{E-}17}$ | $0.489^*_{\pm2.78\text{E-}17}$ |
| | ReMasker | $\underline{0.965}^*_{\pm5.55\text{E-}17}$ | $0.468^*_{\pm1.39\text{E-}17}$ | $0.922^*_{\pm1.85\text{E-}17}$ | $0.762^*_{\pm1.39\text{E-}17}$ | $0.479^*_{\pm1.85\text{E-}17}$ |
| | **NewImp** | $\mathbf{0.967}_{\pm4.93\text{E-}17}$ | $\mathbf{0.494}_{\pm4.32\text{E-}17}$ | $\mathbf{0.934}_{\pm4.93\text{E-}17}$ | $\mathbf{0.839}_{\pm2.47\text{E-}17}$ | $\mathbf{0.495}_{\pm2.08\text{E-}17}$ |
| MNAR | CSDI_T | $0.658^*_{\pm4.16\text{E-}17}$ | $0.078^*_{\pm5.20\text{E-}18}$ | $0.608^*_{\pm0.00\text{E}0}$ | $0.647^*_{\pm0.00\text{E}0}$ | $0.440^*_{\pm6.94\text{E-}18}$ |
| | MissDiff | $0.800^*_{\pm2.47\text{E-}17}$ | $0.322^*_{\pm2.47\text{E-}17}$ | $0.884^*_{\pm3.70\text{E-}17}$ | $0.749^*_{\pm4.93\text{E-}17}$ | $0.480^*_{\pm1.85\text{E-}17}$ |
| | GAIN | $0.963^*_{\pm6.17\text{E-}17}$ | $\underline{0.475}^*_{\pm1.85\text{E-}17}$ | $0.925^*_{\pm3.70\text{E-}17}$ | $0.837^*_{\pm6.17\text{E-}17}$ | $0.493_{\pm2.47\text{E-}17}$ |
| | MIRACLE | $0.930^*_{\pm4.93\text{E-}17}$ | $0.457^*_{\pm3.08\text{E-}17}$ | $0.721^*_{\pm3.70\text{E-}17}$ | $0.777^*_{\pm4.93\text{E-}17}$ | $0.481^*_{\pm1.85\text{E-}17}$ |
| | MIWAE | $0.961^*_{\pm4.16\text{E-}17}$ | $0.437^*_{\pm6.94\text{E-}18}$ | $0.917^*_{\pm2.78\text{E-}17}$ | $\underline{0.839}^*_{\pm1.39\text{E-}17}$ | $\underline{0.495}^*_{\pm1.39\text{E-}17}$ |
| | Sink | $0.946^*_{\pm1.39\text{E-}17}$ | $0.427^*_{\pm4.16\text{E-}17}$ | $0.882^*_{\pm6.94\text{E-}17}$ | $0.781^*_{\pm1.39\text{E-}17}$ | $0.469^*_{\pm2.78\text{E-}17}$ |
| | TDM | $0.962^*_{\pm6.94\text{E-}17}$ | $0.469^*_{\pm2.08\text{E-}17}$ | $0.927^*_{\pm2.78\text{E-}17}$ | $0.773^*_{\pm4.16\text{E-}17}$ | $0.489^*_{\pm3.47\text{E-}17}$ |
| | ReMasker | $\underline{0.965}^*_{\pm2.78\text{E-}17}$ | $0.458_{\pm3.47\text{E-}17}$ | $\underline{0.929}^*_{\pm5.55\text{E-}17}$ | $0.771^*_{\pm2.78\text{E-}17}$ | $0.478^*_{\pm2.78\text{E-}17}$ |
| | **NewImp** | $\mathbf{0.969}_{\pm4.93\text{E-}17}$ | $\mathbf{0.482}_{\pm2.47\text{E-}17}$ | $\mathbf{0.943}_{\pm4.93\text{E-}17}$ | $\mathbf{0.847}_{\pm3.70\text{E-}17}$ | $\mathbf{0.497}_{\pm4.86\text{E-}17}$ |
| Ground Truth | | $0.985_{\pm9.87\text{E-}17}$ | $0.700_{\pm1.23\text{E-}17}$ | $0.989_{\pm0.00\text{E}0}$ | $0.908_{\pm1.23\text{E-}17}$ | $0.566_{\pm2.78\text{E-}17}$ |

*Kindly Note*: The best results are **bolded** and the second best results are underliend. "*" marks the results that NewImp significantly outperform with $p$-value$< 0.05$ over paired samples $t$-test.

Moreover, we also conduct downstream regression task comparisons as detailed in this subsection. Initially, we evaluate the regression performance on imputed data using the following protocol: 1) Selection of datasets with continuous outcome variables. 2) After imputation, we train a support vector regression model equipped with an RBF kernel and an automatic kernel coefficient. We assess the model's performance using 5-fold cross-validation. In this procedure, we report both the mean and standard deviation of the mean squared errors (MSE) and mean absolute error (MAE) across 10 runs with different random seeds. 3) Additionally, we include the MSE and MAE on ground-truth data for reference. The comparative results are presented in Table F.9. As indicated in these results, the NewImp approach consistently outperforms most of the baseline models, further validating its superiority.

Table F.9: Comparison results on the regression task with 30% missing rate.

| Model | CC | | | | | |
|---|---|---|---|---|---|---|
| | MAR | | MCAR | | MNAR | |
| | MAE | MSE | MAE | MSE | MAE | MSE |
| CSDI_T | $1.41\text{E}1^*_{\pm 0.00\text{E}0}$ | $3.07\text{E}2^*_{\pm 5.60\text{E}2}$ | $1.41\text{E}1^*_{\pm 0.00\text{E}0}$ | $3.07\text{E}2^*_{\pm 5.64\text{E}2}$ | $1.41\text{E}1^*_{\pm 0.00\text{E}0}$ | $3.07\text{E}2^*_{\pm 5.59\text{E}2}$ |
| MissDiff | $1.25\text{E}1^*_{\pm 5.92\text{E-}16}$ | $2.31\text{E}2^*_{\pm 3.53\text{E}2}$ | $1.27\text{E}1^*_{\pm 5.92\text{E-}16}$ | $2.44\text{E}2^*_{\pm 3.88\text{E}2}$ | $1.27\text{E}1^*_{\pm 3.95\text{E-}16}$ | $2.45\text{E}2^*_{\pm 3.90\text{E}2}$ |
| GAIN | $\underline{1.23\text{E}1}_{\pm 7.89\text{E-}16}$ | $2.24\text{E}2^*_{\pm 3.60\text{E}2}$ | $1.26\text{E}1^*_{\pm 3.95\text{E-}16}$ | $2.40\text{E}2^*_{\pm 3.90\text{E}2}$ | $1.68\text{E}1^*_{\pm 3.95\text{E-}16}$ | $2.38\text{E}2^*_{\pm 3.95\text{E}2}$ |
| MIRACLE | $1.59\text{E}1^*_{\pm 5.92\text{E-}16}$ | $3.51\text{E}2^*_{\pm 3.51\text{E}2}$ | $1.60\text{E}1^*_{\pm 5.92\text{E-}16}$ | $3.71\text{E}2^*_{\pm 3.71\text{E}2}$ | $1.64\text{E}1^*_{\pm 7.89\text{E-}16}$ | $3.80\text{E}2^*_{\pm 3.80\text{E}2}$ |
| MIWAE | $1.23\text{E}1_{\pm 2.22\text{E-}16}$ | $\underline{2.23\text{E}2}^*_{\pm 3.63\text{E}2}$ | $1.27\text{E}1^*_{\pm 2.22\text{E-}16}$ | $2.43\text{E}2^*_{\pm 3.98\text{E}2}$ | $\underline{1.27\text{E}1}_{\pm 2.22\text{E-}16}$ | $2.42\text{E}2_{\pm 4.05\text{E}2}$ |
| Sink | $1.58\text{E}1^*_{\pm 2.22\text{E-}16}$ | $3.48\text{E}2^*_{\pm 3.48\text{E}2}$ | $1.65\text{E}1^*_{\pm 8.88\text{E-}16}$ | $3.85\text{E}2^*_{\pm 3.85\text{E}2}$ | $1.66\text{E}1^*_{\pm 4.44\text{E-}16}$ | $3.87\text{E}2^*_{\pm 3.87\text{E}2}$ |
| TDM | $1.58\text{E}1_{\pm 4.44\text{E-}16}$ | $\mathbf{2.19\text{E}2}_{\pm 3.49\text{E}2}$ | $\underline{1.26\text{E}1}^*_{\pm 6.66\text{E-}16}$ | $\underline{2.38\text{E}2}^*_{\pm 3.89\text{E}2}$ | $1.66\text{E}1_{\pm 6.66\text{E-}16}$ | $\mathbf{2.37\text{E}2}_{\pm 3.86\text{E}2}$ |
| ReMasker | $1.58\text{E}1^*_{\pm 2.22\text{E-}16}$ | $3.44\text{E}2^*_{\pm 3.44\text{E}2}$ | $1.62\text{E}1^*_{\pm 4.44\text{E-}16}$ | $3.69\text{E}2^*_{\pm 3.69\text{E}2}$ | $1.64\text{E}1^*_{\pm 8.88\text{E-}16}$ | $3.72\text{E}2^*_{\pm 3.72\text{E}2}$ |
| **NewImp** | $\mathbf{1.22\text{E}1}_{\pm 1.18\text{E-}15}$ | $2.37\text{E}2_{\pm 3.49\text{E}2}$ | $\mathbf{1.24\text{E}1}_{\pm 3.95\text{E-}16}$ | $\mathbf{2.37\text{E}2}_{\pm 3.78\text{E}2}$ | $\mathbf{1.26\text{E}1}_{\pm 7.89\text{E-}16}$ | $\underline{2.37\text{E}2}_{\pm 3.93\text{E}2}$ |
| Ground Truth | $1.01\text{E}1_{\pm 5.92\text{E-}16}$ | $1.57\text{E}2_{\pm 3.44\text{E}2}$ | $9.83\text{E}0_{\pm 1.97\text{E-}16}$ | $1.55\text{E}2_{\pm 4.23\text{E}2}$ | $1.04\text{E}1_{\pm 1.97\text{E-}16}$ | $1.68\text{E}2_{\pm 4.27\text{E}2}$ |

*Kindly Note*: The best results are **bolded** and the second best results are underliend. "*" marks the results that NewImp significantly outperform with $p$-value$< 0.05$ over paired samples $t$-test.

## F.7 Baseline Comparison Vary Different Missing Rates and Scenarios

In this section, we further present the extended analysis of model performance across varying missing data rates, as detailed in Tables G.1, G.3, G.5 and G.7, and the corresponding standard deviation error results are presented in Tables G.2, G.4, G.6 and G.8. From the comparison results, it can be seen that our NewImp approach generally perform well compared to most of baseline models. This phenomenon reflects that the proposed NewImp approach is robust to various missing rates, and further proves its applicability.

# Appendix G   Limitations & Future Directions and Broader Impact

## G.1   Limitations & Future Directions

The limitations and future research directions of this work can be summarized as follows:

- **Utilization of Kernel Function:** During the derivation of the velocity field, we employ RKHS to ensure implementation easiness. However, this regularization term may impose restrictions on the velocity field's direction, potentially limiting imputation accuracy in high-dimensional settings. Additionally, the computational complexity tends to scale quadratically with dataset size increases. Exploring alternative regularization terms [13] to replace RKHS presents a promising direction.

- **Training of Score Function:** As discussed in Section F.4, the runtime of NewImp is predominantly governed by the DSM function. Investigating techniques to reduce the training costs of this part, such as employing sliced score matching [49], represents an intriguing area for future exploration.

- **Wasserstein Gradient Flow Framework:** The WGF framework currently operates as a first-order system where each sample is equally weighted. A critical advancement would be the incorporation of second-order systems, such as Hamiltonian dynamics [55, 61], and other gradient flows like Fisher-Rao gradient flow [69] that assign variable weights to samples. These adaptations aim to decrease computational times inherently.

- $\mathbb{R}^{\mathrm{D}}$ **Support Assumption:** In our manuscript, for ease of derivation, we assume that the distribution we model has support on the real number domain $\mathbb{R}^{\mathrm{D}}$, which limits the direct application of NewImp to tabular data with categorical variables. This limitation can be alleviated by employing the mirror descent approach. Specifically, for a categorical variable with $\mathrm{D}$ categories, the distribution belongs to the Dirichlet distribution whose support lies on the simplex $\Delta^{\mathrm{D}-1}$. On this basis, we can define the Bregman function as the entropy function: $\Psi(\boldsymbol{X}) \coloneqq \sum_{j=1}^{\mathrm{D}} (\boldsymbol{X}_j \log \boldsymbol{X}_j - \boldsymbol{X}_j)$ and apply mirror descent using this Bregman function to handle the categorical domain effectively. Notably, similar approaches have been successfully applied in works focusing on constrained domain sampling, as exemplified in [48]. We have implemented a comparable scheme in Section 3.1 and Appendix B.

## G.2 Broader Impact Statement

MDI and DMs are pivotal areas within machine learning, each boasting a wide array of real-world applications. While numerous applications exist, this paper does not single out any specific ones; instead, it focuses on addressing fundamental challenges in these fields. This study significantly advances the application of DMs for MDI by tackling prevalent issues such as inaccurate imputation and challenging training processes. We believe that the insights garnered here can be applied to related domains, such as probabilistic time-series forecasting and image inpainting, where accuracy is often more critical than diversity in results. A common challenge across these domains is the nuanced need for precision over variety, which can lead to overlooked opportunities in model application and development. Our proposed method provides a fresh perspective on these tasks through an optimization lens. It evaluates the appropriateness of directly applying existing diffusion models to these tasks and, where necessary, proposes the derivation of novel algorithms. This approach not only enhances the understanding of the underlying mechanisms but also paves the way for more targeted and effective solutions in the future.

Table G.1: Performance of MAE and WASS metrics at 10% missing rate.

| Scenario | Model | BT MAE | BT WASS | BCD MAE | BCD WASS | CC MAE | CC WASS | CBV MAE | CBV WASS | IS MAE | IS WASS | PK MAE | PK WASS | QB MAE | QB WASS | WQW MAE | WQW WASS |
|---|---|---|---|---|---|---|---|---|---|---|---|---|---|---|---|---|---|
| MAR | CSDI_T | 1.22* | 6.09* | 1.04* | 23.73* | 0.93* | 3.43* | 0.83* | 4.06* | 0.70* | 16.73* | 1.36* | 23.27* | 0.76* | 24.16* | 0.83* | 5.06* |
| | MissDiff | 1.08* | 3.53* | 1.02* | 22.26* | 0.92* | 1.55* | 0.84* | 5.74* | 0.70* | 12.43* | 1.25* | 21.05* | 0.74* | 28.80* | 0.82* | 7.58* |
| | GAIN | 0.97* | 1.11* | 0.51* | 0.87* | 0.84* | 0.64* | 0.71* | 0.50* | 0.43 | 0.56 | 0.66 | 1.31 | 0.55 | 1.69* | 0.70* | 0.63* |
| | MIRACLE | 0.65* | 0.41 | 0.52* | 0.93* | 0.35* | 0.14 | 0.50* | 0.26* | 3.87* | 19.88* | 4.46* | 23.51* | **0.45*** | **1.23*** | 0.46* | 0.33* |
| | MIWAE | 0.74* | 0.87* | 0.60* | 1.18* | 0.74* | 0.51* | 0.83* | 0.67* | 0.45* | 0.80* | 0.76* | 1.78* | 0.63 | 2.84 | 0.61* | 0.56* |
| | Sink | 1.09* | 1.60* | 1.00* | 2.53* | 0.92* | 0.67* | 0.84* | 0.67* | 0.71* | 1.03* | 1.25* | 3.52* | 0.75* | 2.69 | 0.82* | 0.81* |
| | TDM | 1.02* | 1.49* | 0.89* | 2.22* | 0.78* | 0.54* | 0.73* | 0.54* | 0.57* | 0.78 | 1.14* | 3.26* | 0.68* | 2.61 | 0.77* | 0.75* |
| | ReMasker | 0.61 | 1.00 | 0.64* | 1.04* | 1.02* | 0.85* | 0.47* | 0.23* | 0.65* | 1.01* | 0.77* | 1.29 | 0.76* | 2.31* | 0.61* | 0.41* |
| | **NewImp** | **0.40** | **0.17** | **0.31** | **0.43** | **0.23** | **0.11** | **0.22** | **0.07** | **0.33** | **0.50** | **0.46** | **0.74** | 0.56 | 3.30 | **0.37** | **0.29** |
| MCAR | CSDI_T | 0.71* | 1.99* | 0.72* | 16.40* | 0.86* | 2.58* | 0.81* | 3.69* | 0.75* | 16.01* | 0.73* | 12.20* | 0.56* | 20.77* | 0.78* | 4.02* |
| | MissDiff | 0.70* | 1.67* | 0.72* | 14.87* | 0.86* | 1.31* | 0.81* | 3.57* | 0.74* | 13.37* | 0.72* | 17.05* | 0.56* | 19.35* | 0.76* | 5.44* |
| | GAIN | 0.66* | 0.24* | 0.36* | 0.44* | 0.76* | 0.47* | 0.69* | 0.54* | 0.53* | 1.14 | 0.38* | 0.40 | 0.39* | 1.06 | 0.73* | 0.61* |
| | MIRACLE | 0.44* | 0.10 | 0.36* | 0.43* | 0.38* | 0.16* | 0.53* | 0.33* | 3.68* | 26.51* | 3.78* | 18.22* | **0.29*** | **0.66*** | 0.47* | 0.33* |
| | MIWAE | 0.49* | 0.13* | 0.46* | 0.83* | 0.71* | 0.43* | 0.79* | 0.68* | 0.60* | 1.81* | 0.50* | 0.74* | 0.43* | 1.81* | 0.63* | 0.51* |
| | Sink | 0.70* | 0.30* | 0.75* | 1.60* | 0.83* | 0.53* | 0.81* | 0.72* | 0.75* | 1.69* | 0.75* | 1.24* | 0.59* | 1.94* | 0.76* | 0.66* |
| | TDM | 0.62* | 0.24* | 0.60* | 1.22* | 0.70* | 0.43* | 0.71* | 0.59* | 0.62* | 1.36* | 0.62* | 0.99* | 0.48* | 1.60* | 0.72* | 0.61* |
| | ReMasker | 0.39 | 0.11 | 0.46* | 0.68* | 1.04* | 0.83* | 0.48* | 0.28* | 0.70* | 1.74* | 0.55* | 0.80* | 0.69* | 2.94 | 0.53* | 0.39* |
| | **NewImp** | **0.38** | **0.09** | **0.19** | **0.18** | **0.27** | **0.11** | **0.23** | **0.10** | **0.42** | **1.01** | **0.25** | **0.28** | 0.30 | 1.06 | **0.40** | **0.30** |
| MNAR | CSDI_T | 0.95* | 3.52* | 0.88* | 17.87* | 0.86* | 2.93* | 0.83* | 3.97* | 0.75* | 16.50* | 1.01* | 13.26* | 0.68* | 21.44* | 0.79* | 4.28* |
| | MissDiff | 0.91* | 2.47* | 0.88* | 16.20* | 0.85* | 1.19* | 0.84* | 3.70* | 0.75* | 13.39* | 0.99* | 19.32* | 0.66* | 23.29* | 0.77* | 5.95* |
| | GAIN | 0.86* | 0.61* | 0.42* | 0.70* | 0.79* | 0.55* | 0.69* | 0.54* | 0.51* | 1.08 | 0.51* | 0.92* | 0.47* | 1.69 | 0.73* | 0.70* |
| | MIRACLE | 0.55* | 0.21* | 0.45* | 1.29 | 0.37* | 0.16* | 0.53* | 0.34* | 3.82* | 28.83* | 4.19* | 23.82* | **0.39*** | **1.26*** | 0.49* | 0.40* |
| | MIWAE | 0.65* | 0.42 | 0.53* | 1.16* | 0.69* | 0.46* | 0.79* | 0.70* | 0.53 | 1.39* | 0.62* | 1.33* | 0.54* | 2.94* | 0.64* | 0.60* |
| | Sink | 0.87* | 0.76* | 0.88* | 2.48* | 0.86* | 0.62* | 0.82* | 0.75* | 0.73* | 1.62* | 0.99* | 2.70* | 0.68* | 3.29* | 0.78* | 0.75* |
| | TDM | 0.80* | 0.71* | 0.75* | 2.11* | 0.74* | 0.50* | 0.73* | 0.63* | 0.59* | 1.24* | 0.86* | 2.39* | 0.59* | 2.63 | 0.73* | 0.71* |
| | ReMasker | 0.49 | 0.30 | 0.53* | 0.98* | 1.02* | 0.88* | 0.48* | 0.29* | 0.66* | 1.61* | 0.67* | 1.25* | 0.64* | 2.72 | 0.56* | 0.48* |
| | **NewImp** | **0.45** | **0.17** | **0.24** | **0.35** | **0.26** | **0.12** | **0.24** | **0.10** | **0.39** | **0.94** | **0.37** | **0.61** | 0.43 | 2.27 | **0.42** | **0.37** |

*Kindly Note*: The best results are **bolded** and the second best results are underlined. "*" marks the results that NewImp significantly outperform with $p$-value$< 0.05$ over paired samples $t$-test.

Table G.2: Standard deviation of MAE and WASS metrics at 10% missing rate.

| Scenario | Model | BT MAE | BT WASS | BCD MAE | BCD WASS | CC MAE | CC WASS | CBV MAE | CBV WASS | IS MAE | IS WASS | PK MAE | PK WASS | QB MAE | QB WASS | WQW MAE | WQW WASS |
|---|---|---|---|---|---|---|---|---|---|---|---|---|---|---|---|---|---|
| MAR | CSDI_T | 8.40E-2 | 4.70E-1 | 5.73E-2 | 4.03E0 | 7.79E-2 | 2.05E-1 | 2.22E-2 | 1.96E0 | 2.97E-2 | 1.43E0 | 6.85E-2 | 3.39E0 | 4.30E-2 | 3.83E-1 | 2.67E-2 | 1.08E-1 |
| | MissDiff | 4.34E-2 | 3.24E-1 | 5.70E-2 | 4.26E0 | 3.51E-2 | 3.77E-1 | 3.71E-2 | 1.50E0 | 4.67E-2 | 7.05E-1 | 3.18E-2 | 3.35E0 | 3.96E-2 | 7.29E0 | 2.87E-2 | 1.59E0 |
| | GAIN | 1.44E-1 | 3.32E-1 | 5.59E-2 | 1.85E-1 | 3.78E-2 | 4.78E-2 | 1.18E-2 | 2.58E-2 | 5.25E-2 | 7.59E-2 | 1.57E-1 | 7.22E-1 | 2.44E-2 | 1.92E-1 | 1.25E-2 | 2.19E-2 |
| | MIRACLE | 6.55E-2 | 1.12E-1 | 1.19E-2 | 1.38E-1 | 1.46E-2 | 1.27E-2 | 1.49E-2 | 1.95E-2 | 1.90E-1 | 1.52E0 | 1.99E-1 | 1.30E0 | 1.08E-2 | 7.11E-2 | 2.12E-3 | 1.85E-2 |
| | MIWAE | 8.87E-2 | 2.19E-1 | 2.99E-2 | 1.77E-1 | 4.67E-2 | 4.99E-2 | 5.67E-2 | 9.82E-2 | 1.96E-2 | 5.08E-2 | 8.45E-2 | 4.42E-1 | 4.26E-2 | 2.86E-1 | 2.15E-2 | 3.69E-2 |
| | Sink | 5.28E-2 | 1.26E-1 | 5.66E-2 | 2.93E-1 | 3.16E-2 | 5.04E-2 | 2.83E-2 | 5.79E-2 | 4.53E-2 | 8.88E-2 | 4.67E-2 | 2.93E-1 | 3.38E-2 | 1.84E-1 | 3.13E-2 | 4.13E-2 |
| | TDM | 8.42E-2 | 1.98E-1 | 3.66E-2 | 2.21E-1 | 4.75E-2 | 6.48E-2 | 2.51E-2 | 5.68E-2 | 3.65E-2 | 7.34E-2 | 5.81E-2 | 2.97E-1 | 3.28E-2 | 1.90E-1 | 3.16E-2 | 3.87E-2 |
| | ReMasker | 2.33E-1 | 8.46E-1 | 5.57E-2 | 2.07E-1 | 1.83E-1 | 2.35E-1 | 2.64E-2 | 1.98E-1 | 8.75E-2 | 1.61E-1 | 1.19E-1 | 3.01E-1 | 5.70E-2 | 1.46E-1 | 3.16E-2 | 3.40E-2 |
| | **NewImp** | 5.63E-2 | 5.33E-2 | 2.25E-2 | 9.03E-2 | 2.36E-2 | 2.73E-2 | 9.44E-3 | 8.75E-3 | 1.22E-2 | 2.57E-2 | 4.05E-2 | 1.34E-1 | 2.90E-2 | 5.76E-1 | 1.70E-2 | 1.39E-2 |
| MCAR | CSDI_T | 6.90E-2 | 4.06E-1 | 1.58E-2 | 1.65E0 | 1.51E-2 | 9.41E-2 | 2.33E-2 | 3.58E-2 | 9.82E-3 | 1.01E0 | 1.31E-2 | 1.53E0 | 9.31E-3 | 8.50E-1 | 1.03E-2 | 2.82E-2 |
| | MissDiff | 4.96E-2 | 7.60E-1 | 6.70E-3 | 1.04E0 | 2.50E-2 | 3.11E-1 | 2.36E-2 | 1.15E0 | 1.57E-2 | 1.69E-1 | 2.17E-2 | 4.25E0 | 8.41E-3 | 3.11E0 | 5.96E-3 | 8.13E-1 |
| | GAIN | 4.38E-2 | 4.20E-2 | 8.76E-3 | 5.12E-2 | 1.41E-2 | 1.22E-2 | 2.65E-2 | 2.09E-2 | 1.92E-2 | 5.98E-2 | 2.05E-2 | 3.80E-2 | 3.98E-3 | 5.01E-2 | 2.27E-2 | 2.88E-2 |
| | MIRACLE | 3.17E-2 | 2.13E-2 | 1.59E-2 | 6.51E-2 | 1.63E-2 | 1.03E-2 | 8.49E-3 | 1.19E-2 | 3.40E-2 | 1.47E-1 | 3.42E-2 | 8.95E-1 | 4.42E-3 | 6.60E-2 | 3.87E-3 | 7.14E-3 |
| | MIWAE | 6.09E-2 | 1.90E-2 | 1.18E-2 | 8.69E-2 | 1.70E-2 | 2.15E-2 | 4.14E-2 | 6.57E-2 | 2.40E-2 | 1.98E-1 | 1.92E-2 | 6.49E-2 | 1.49E-2 | 8.14E-2 | 7.89E-3 | 1.34E-2 |
| | Sink | 3.16E-2 | 2.47E-2 | 2.29E-2 | 1.47E-1 | 6.26E-3 | 8.34E-3 | 1.84E-2 | 2.20E-2 | 8.12E-3 | 4.84E-2 | 2.87E-2 | 1.23E-1 | 8.64E-3 | 7.37E-2 | 5.37E-3 | 1.77E-2 |
| | TDM | 3.21E-2 | 2.86E-2 | 2.17E-2 | 1.37E-1 | 6.99E-3 | 1.19E-2 | 1.53E-2 | 2.11E-2 | 8.58E-3 | 4.32E-2 | 2.43E-2 | 9.17E-2 | 6.84E-3 | 5.71E-2 | 5.76E-3 | 6.96E-3 |
| | ReMasker | 4.99E-2 | 2.79E-2 | 3.02E-2 | 1.24E-1 | 1.78E-1 | 2.20E-1 | 1.39E-2 | 1.17E-2 | 4.38E-2 | 1.14E-1 | 6.17E-2 | 1.34E-1 | 1.89E-1 | 1.37E0 | 3.40E-2 | 2.15E-2 |
| | **NewImp** | 2.82E-2 | 2.16E-2 | 7.19E-3 | 2.94E-2 | 7.99E-3 | 8.76E-3 | 9.89E-3 | 1.11E-2 | 1.05E-2 | 6.14E-2 | 4.15E-3 | 2.23E-2 | 8.38E-3 | 1.33E-1 | 1.01E-3 | 6.86E-3 |
| MNAR | CSDI_T | 1.47E-1 | 4.94E-1 | 1.71E-2 | 2.27E0 | 3.58E-2 | 1.73E-1 | 1.79E-2 | 1.50E-1 | 1.23E-2 | 1.89E0 | 7.42E-2 | 2.65E0 | 1.32E-2 | 4.92E-1 | 1.43E-2 | 1.19E-1 |
| | MissDiff | 1.19E-1 | 2.63E-1 | 2.16E-2 | 2.32E0 | 1.80E-2 | 1.81E-1 | 2.77E-2 | 9.92E-1 | 1.54E-2 | 6.34E-1 | 5.43E-2 | 4.20E0 | 6.09E-3 | 4.98E0 | 1.64E-2 | 8.58E-1 |
| | GAIN | 7.02E-2 | 2.01E-1 | 1.90E-2 | 6.29E-2 | 4.72E-2 | 6.21E-2 | 3.69E-2 | 6.55E-2 | 2.32E-2 | 5.24E-2 | 2.17E-2 | 6.10E-2 | 1.39E-2 | 2.34E-1 | 3.08E-2 | 3.76E-2 |
| | MIRACLE | 5.13E-2 | 7.64E-2 | 4.21E-2 | 8.97E-1 | 9.90E-3 | 6.04E-2 | 8.72E-3 | 1.04E-2 | 1.76E-1 | 1.36E0 | 9.38E-2 | 1.00E0 | 7.89E-3 | 1.91E-1 | 6.14E-3 | 2.36E-2 |
| | MIWAE | 1.03E-1 | 2.46E-1 | 1.31E-2 | 8.95E-2 | 1.94E-2 | 3.61E-2 | 4.34E-2 | 7.25E-2 | 4.83E-2 | 8.55E-2 | 2.57E-2 | 1.70E-1 | 4.28E-2 | 3.40E-1 | 1.16E-2 | 2.48E-2 |
| | Sink | 1.10E-1 | 3.22E-1 | 1.49E-2 | 7.35E-2 | 2.26E-2 | 3.63E-2 | 2.57E-2 | 5.42E-2 | 9.10E-3 | 4.04E-2 | 3.37E-2 | 1.57E-1 | 9.34E-3 | 2.04E-1 | 1.77E-2 | 3.95E-2 |
| | TDM | 1.28E-1 | 3.40E-1 | 1.25E-2 | 6.81E-2 | 2.76E-2 | 3.32E-2 | 2.76E-2 | 5.37E-2 | 1.66E-2 | 2.48E-2 | 2.62E-2 | 1.15E-1 | 2.01E-2 | 1.93E-1 | 9.26E-3 | 2.65E-2 |
| | ReMasker | 2.41E-2 | 5.68E-2 | 6.82E-2 | 1.70E-1 | 1.07E-1 | 1.70E-1 | 2.33E-2 | 3.57E-2 | 2.46E-2 | 7.26E-2 | 1.42E-2 | 9.66E-2 | 1.93E-2 | 3.59E-1 | 3.33E-2 | 2.70E-2 |
| | **NewImp** | 5.64E-2 | 7.75E-2 | 4.16E-3 | 2.50E-2 | 9.49E-3 | 1.32E-2 | 1.18E-2 | 1.16E-2 | 9.58E-3 | 3.64E-2 | 2.08E-2 | 9.91E-2 | 1.70E-2 | 3.24E-1 | 9.42E-3 | 2.73E-2 |

Table G.3: Performance of MAE and WASS metrics at 20% missing rate.

| Scenario | Model | BT | | BCD | | CC | | CBV | | IS | | PK | | QB | | WQW | |
|---|---|---|---|---|---|---|---|---|---|---|---|---|---|---|---|---|---|
| | | MAE | WASS | MAE | WASS | MAE | WASS | MAE | WASS | MAE | WASS | MAE | WASS | MAE | WASS | MAE | WASS |
| MAR | CSDI_T | 0.99 * | 4.07 * | 0.98 * | 19.69 * | 0.89 * | 2.97 * | 0.86 * | 4.09 * | 0.69 * | 16.70 * | 1.07 * | 13.54 * | 0.70 * | 20.67 * | 0.78 * | 4.36 * |
| | MissDiff | 0.91 * | 2.19 * | 0.96 * | 18.37 * | 0.87 * | 1.37 * | 0.87 * | 5.71 * | 0.70 * | 13.36 * | 1.05 * | 17.50 * | 0.69 * | 24.45 * | 0.78 * | 6.64 * |
| | GAIN | 1.02 * | 1.11 | 0.53 * | 1.35 * | 0.74 * | 0.58 * | 0.71 * | 0.63 * | 0.48 * | 1.01 | 0.55 * | 1.07 | 0.59 * | 3.02 | 0.75 * | 0.82 * |
| | MIRACLE | 0.69 * | 0.46 | 0.47 * | 1.14 * | 0.38 * | 0.18 * | 0.54 * | 0.37 * | 3.80 * | 30.49 * | 3.99 * | 23.48 * | 0.45 * | **1.91 *** | 0.51 * | 0.48 |
| | MIWAE | 0.68 * | 0.61 | 0.56 * | 1.51 * | 0.68 * | 0.52 * | 0.84 * | 0.82 * | 0.51 * | 1.39 * | 0.67 * | 1.56 * | 0.59 * | 3.81 * | 0.65 * | 0.67 * |
| | Sink | 0.93 * | 1.10 * | 0.96 * | 3.45 * | 0.88 * | 0.72 * | 0.88 * | 0.89 * | 0.73 * | 1.68 * | 1.04 * | 3.21 * | 0.70 * | 3.49 * | 0.78 * | 0.89 * |
| | TDM | 0.88 * | 1.06 * | 0.87 * | 3.07 * | 0.77 * | 0.61 * | 0.78 * | 0.74 * | 0.59 * | 1.31 * | 0.95 * | 2.95 * | 0.63 * | 3.32 * | 0.75 * | 0.84 * |
| | ReMasker | **0.51** | **0.42** | 0.52 * | 1.16 * | nan | nan | 0.50 * | 0.32 * | 0.60 * | 1.49 * | 0.65 * | 1.39 * | 0.76 | 3.98 | 0.53 * | 0.51 * |
| | **NewImp** | 0.55 | 0.53 | **0.31** | **0.68** | **0.28** | **0.16** | **0.27** | **0.13** | **0.35** | **0.83** | **0.42** | **0.78** | 0.51 | 2.91 | **0.45** | **0.46** |
| MCAR | CSDI_T | 0.74 * | 2.05 * | 0.73 * | 16.83 * | 0.85 * | 2.63 * | 0.82 * | 3.73 * | 0.76 * | 15.93 * | 0.74 * | 11.85 * | 0.57 * | 20.29 * | 0.78 * | 4.03 * |
| | MissDiff | 0.73 * | 1.64 * | 0.74 * | 14.08 * | 0.84 * | 1.25 * | 0.82 * | 3.71 * | 0.74 * | 13.09 * | 0.73 * | 16.22 * | 0.56 * | 22.27 * | 0.76 * | 5.46 * |
| | GAIN | 0.70 * | 0.31 * | 0.36 * | 0.86 * | 0.77 * | 0.60 * | 0.69 * | 0.72 * | 0.53 * | 2.27 * | 0.39 * | 0.76 * | 0.41 * | 2.13 | 0.71 * | 0.82 * |
| | MIRACLE | 0.48 * | 0.12 | 0.40 * | 1.05 * | 0.44 * | 0.23 * | 0.56 * | 0.50 * | 3.37 * | 44.75 * | 3.60 * | 31.60 * | **0.32** | **1.46 *** | 0.52 * | 0.53 * |
| | MIWAE | 0.53 * | 0.18 | 0.48 * | 1.68 * | 0.73 * | 0.55 * | 0.81 * | 0.95 * | 0.62 * | 3.18 * | 0.48 * | 1.26 * | 0.47 * | 3.92 * | 0.65 * | 0.73 * |
| | Sink | 0.73 * | 0.40 * | 0.75 * | 2.99 * | 0.84 * | 0.68 * | 0.82 * | 0.97 * | 0.75 * | 3.30 * | 0.73 * | 2.14 * | 0.60 * | 3.81 * | 0.76 * | 0.94 * |
| | TDM | 0.67 * | 0.35 * | 0.62 * | 2.37 * | 0.74 * | 0.56 * | 0.75 * | 0.84 * | 0.64 * | 2.73 * | 0.61 * | 1.72 * | 0.50 * | 3.31 * | 0.73 * | 0.88 * |
| | ReMasker | **0.42** | **0.10** | 0.41 * | 1.22 * | nan | nan | 0.50 * | 0.41 * | 0.62 * | 2.87 * | 0.51 * | 1.36 * | 0.56 * | 4.09 | 0.54 * | 0.55 |
| | **NewImp** | 0.44 | 0.14 | **0.22** | **0.44** | **0.36** | **0.20** | **0.31** | **0.22** | **0.41** | **1.92** | **0.28** | **0.59** | 0.32 | 2.27 | **0.47** | **0.50** |
| MNAR | CSDI_T | 0.85 * | 2.50 * | 0.86 * | 17.08 * | 0.84 * | 2.70 * | 0.83 * | 3.84 * | 0.74 * | 15.92 * | 0.90 * | 11.70 * | 0.64 * | 20.01 * | 0.78 * | 4.10 * |
| | MissDiff | 0.80 * | 1.63 * | 0.85 * | 15.69 * | 0.85 * | 1.18 * | 0.83 * | 4.40 * | 0.73 * | 13.54 * | 0.89 * | 16.08 * | 0.63 * | 21.83 * | 0.76 * | 5.69 * |
| | GAIN | 0.80 * | 0.48 * | 0.43 * | 1.46 * | 0.72 * | 0.57 * | 0.68 * | 0.74 * | 0.56 * | 2.40 * | 0.55 * | 2.05 * | 0.51 * | 4.08 | 0.71 * | 0.87 * |
| | MIRACLE | 0.54 | 0.19 | 0.48 * | 2.02 | 0.44 * | 0.23 | 0.57 * | 0.54 * | 3.35 * | 42.49 * | 3.85 * | 34.51 * | 0.43 | **2.89** | 0.53 * | 0.57 * |
| | MIWAE | 0.60 * | 0.31 * | 0.53 * | 2.08 * | 0.72 * | 0.56 * | 0.82 * | 1.00 * | 0.57 * | 2.81 * | 0.63 * | 2.32 * | 0.54 * | 5.52 * | 0.65 * | 0.76 * |
| | Sink | 0.77 * | 0.55 * | 0.84 * | 4.13 * | 0.83 * | 0.69 * | 0.82 * | 1.02 * | 0.74 * | 3.19 * | 0.91 * | 3.98 * | 0.67 * | 5.39 * | 0.76 * | 0.99 * |
| | TDM | 0.71 * | 0.49 * | 0.74 * | 3.56 * | 0.73 * | 0.58 * | 0.76 * | 0.90 * | 0.62 * | 2.60 * | 0.81 * | 3.57 * | 0.57 * | 4.95 | 0.73 * | 0.94 * |
| | ReMasker | 0.49 | 0.27 | 0.48 * | 1.56 * | nan | nan | 0.52 * | 0.46 * | 0.55 * | 2.52 * | 0.59 * | 1.84 * | 0.58 * | 4.49 | 0.56 * | 0.62 * |
| | **NewImp** | **0.48** | **0.17** | **0.27** | **0.76** | **0.36** | **0.21** | **0.33** | **0.25** | **0.39** | **1.79** | **0.39** | **1.17** | **0.43** | 4.03 | **0.48** | **0.54** |

*Kindly Note*: The best results are **bolded** and the second best results are underliend. "*" marks the results that NewImp significantly outperform with $p$-value$< 0.05$ over paired samples $t$-test.

Table G.4: Standard deviation of MAE and WASS metrics at 20% missing rate.

| Scenario | Model | BT | | BCD | | CC | | CBV | | IS | | PK | | QB | | WQW | |
|---|---|---|---|---|---|---|---|---|---|---|---|---|---|---|---|---|---|
| | | MAE | WASS | MAE | WASS | MAE | WASS | MAE | WASS | MAE | WASS | MAE | WASS | MAE | WASS | MAE | WASS |
| MAR | CSDI_T | 1.45E-1 | 5.95E-1 | 3.96E-2 | 2.78E0 | 6.42E-2 | 2.35E-1 | 3.95E-3 | 2.24E-1 | 5.04E-3 | 2.03E0 | 9.93E-2 | 3.40E0 | 1.84E-2 | 9.01E-1 | 2.76E-2 | 1.28E-1 |
| | MissDiff | 1.35E-1 | 7.18E-1 | 7.42E-3 | 3.48E0 | 1.84E-2 | 1.67E-1 | 1.38E-2 | 1.02E0 | 1.29E-2 | 8.67E-1 | 3.32E-2 | 3.61E0 | 7.57E-3 | 5.95E0 | 3.28E-2 | 8.13E-1 |
| | GAIN | 3.18E-1 | 5.29E-1 | 5.32E-2 | 2.81E-1 | 3.98E-2 | 5.34E-2 | 2.87E-2 | 6.18E-2 | 4.32E-2 | 1.25E-1 | 2.46E-3 | 3.84E-2 | 5.71E-2 | 3.76E-1 | 6.93E-2 | 9.70E-2 |
| | MIRACLE | 1.44E-1 | 1.75E-1 | 2.22E-2 | 6.20E-2 | 1.89E-2 | 1.43E-2 | 1.49E-2 | 2.46E-2 | 3.01E-1 | 4.73E0 | 3.73E-1 | 3.82E0 | 2.56E-2 | 2.59E-1 | 4.69E-2 | 3.77E-2 |
| | MIWAE | 1.62E-1 | 2.25E-1 | 2.11E-2 | 1.76E-1 | 1.31E-2 | 4.51E-2 | 3.24E-2 | 6.88E-2 | 2.55E-2 | 4.75E-2 | 3.67E-2 | 1.36E-1 | 2.82E-2 | 2.78E-1 | 6.03E-2 | 5.11E-2 |
| | Sink | 1.38E-1 | 2.45E-1 | 9.11E-3 | 1.75E-1 | 3.48E-2 | 7.35E-2 | 1.23E-2 | 3.27E-2 | 3.42E-3 | 4.49E-2 | 3.73E-2 | 2.47E-1 | 1.08E-2 | 3.28E-1 | 3.56E-2 | 6.35E-2 |
| | TDM | 1.53E-1 | 2.59E-1 | 4.48E-3 | 1.16E-1 | 3.30E-2 | 7.34E-2 | 8.41E-3 | 2.25E-2 | 2.85E-2 | 3.93E-2 | 3.31E-2 | 2.15E-1 | 2.29E-2 | 3.44E-1 | 3.24E-2 | 5.11E-2 |
| | ReMasker | 1.27E-1 | 1.32E-1 | 9.08E-2 | 3.42E-1 | 9.42E-2 | 1.75E-1 | 1.90E-2 | 2.42E-2 | 3.78E-2 | 8.97E-2 | 4.59E-2 | 1.29E-1 | 1.54E-1 | 1.38E0 | 4.45E-2 | 4.02E-2 |
| | **NewImp** | 1.33E-1 | 2.16E-1 | 1.29E-2 | 1.18E-1 | 1.87E-2 | 1.23E-2 | 3.36E-3 | 5.46E-3 | 4.84E-3 | 4.37E-2 | 2.55E-2 | 1.49E-1 | 1.98E-2 | 1.76E-1 | 6.04E-2 | 6.49E-2 |
| MCAR | CSDI_T | 4.54E-2 | 5.09E-2 | 1.05E-2 | 1.33E0 | 1.73E-2 | 1.46E-1 | 5.36E-3 | 3.03E-2 | 8.51E-3 | 7.25E-1 | 1.69E-2 | 1.37E0 | 4.54E-3 | 5.07E-1 | 2.54E-3 | 2.50E-2 |
| | MissDiff | 3.37E-2 | 5.02E-1 | 9.14E-3 | 5.36E-1 | 1.75E-2 | 2.94E-1 | 7.44E-3 | 1.06E0 | 9.22E-3 | 9.56E-2 | 1.49E-2 | 2.37E0 | 3.06E-3 | 2.59E0 | 7.27E-3 | 1.13E0 |
| | GAIN | 5.67E-2 | 5.60E-2 | 7.51E-3 | 3.57E-2 | 6.21E-3 | 2.45E-2 | 2.02E-2 | 3.50E-2 | 7.40E-3 | 3.03E-2 | 9.46E-3 | 4.17E-2 | 1.42E-2 | 1.77E-1 | 1.67E-2 | 1.82E-2 |
| | MIRACLE | 2.35E-2 | 2.34E-2 | 1.35E-2 | 4.06E-2 | 5.66E-3 | 6.01E-3 | 5.36E-3 | 1.14E-2 | 2.36E-2 | 1.98E-1 | 1.21E-1 | 1.83E0 | 4.58E-3 | 1.09E-1 | 7.64E-3 | 2.30E-2 |
| | MIWAE | 2.76E-2 | 1.85E-2 | 1.10E-2 | 5.57E-2 | 2.79E-2 | 4.10E-2 | 1.68E-2 | 5.13E-2 | 2.73E-2 | 1.16E-1 | 1.64E-2 | 7.49E-2 | 1.61E-2 | 2.68E-1 | 5.22E-3 | 1.72E-2 |
| | Sink | 2.04E-2 | 2.24E-2 | 5.92E-3 | 7.02E-2 | 1.33E-2 | 2.09E-2 | 1.16E-2 | 3.50E-2 | 2.84E-3 | 5.01E-2 | 8.93E-3 | 8.83E-2 | 7.60E-3 | 2.46E-1 | 6.25E-3 | 2.48E-2 |
| | TDM | 1.48E-2 | 2.87E-2 | 6.57E-3 | 7.20E-2 | 1.42E-2 | 1.93E-2 | 1.38E-2 | 3.52E-2 | 2.61E-3 | 7.96E-3 | 1.84E-2 | 9.92E-2 | 1.01E-2 | 1.93E-1 | 4.91E-3 | 1.32E-2 |
| | ReMasker | 3.25E-2 | 9.47E-3 | 1.41E-2 | 1.29E-1 | 1.07E-1 | 1.96E-1 | 1.09E-2 | 1.09E-2 | 4.00E-2 | 1.63E-1 | 4.79E-2 | 3.06E-1 | 1.13E-1 | 1.56E0 | 4.82E-3 | 1.22E-2 |
| | **NewImp** | 3.25E-2 | 2.44E-2 | 1.42E-2 | 2.83E-2 | 3.12E-3 | 1.66E-2 | 8.22E-3 | 1.95E-1 | 1.13E-2 | 4.72E-2 | 6.48E-3 | 2.83E-2 | 1.97E-3 | 1.52E-1 | 4.82E-3 | 1.22E-2 |
| MNAR | CSDI_T | 8.79E-2 | 3.69E-1 | 2.23E-2 | 1.85E0 | 1.92E-2 | 1.35E-1 | 1.44E-2 | 1.07E-1 | 1.17E-2 | 1.28E0 | 1.33E-2 | 1.89E0 | 5.62E-3 | 6.07E-1 | 7.60E-3 | 3.31E-1 |
| | MissDiff | 5.88E-2 | 7.69E-1 | 8.45E-3 | 1.82E0 | 1.03E-2 | 2.02E-1 | 1.97E-2 | 8.06E-1 | 2.19E-2 | 2.28E-1 | 1.39E-2 | 4.05E0 | 8.29E-3 | 3.70E0 | 8.84E-3 | 2.05E-1 |
| | GAIN | 3.27E-2 | 8.42E-2 | 2.23E-2 | 1.90E-1 | 4.37E-2 | 6.57E-2 | 2.17E-2 | 3.99E-2 | 7.77E-3 | 8.22E-2 | 2.45E-2 | 2.00E-1 | 2.07E-2 | 2.61E-1 | 1.65E-2 | 5.25E-2 |
| | MIRACLE | 2.42E-2 | 2.71E-2 | 5.33E-2 | 8.42E-1 | 1.65E-2 | 1.82E-2 | 2.34E-2 | 3.99E-2 | 7.48E-2 | 2.21E0 | 6.79E-2 | 1.25E0 | 2.25E-2 | 3.69E-1 | 5.58E-3 | 1.39E-2 |
| | MIWAE | 1.04E-2 | 3.27E-2 | 9.20E-3 | 4.25E-2 | 3.77E-2 | 6.09E-2 | 1.96E-2 | 4.44E-2 | 2.51E-2 | 1.70E-1 | 5.47E-2 | 3.37E-1 | 3.40E-2 | 4.35E-1 | 6.92E-3 | 2.52E-2 |
| | Sink | 1.93E-2 | 6.83E-2 | 2.07E-2 | 1.43E-1 | 8.90E-3 | 2.11E-2 | 2.06E-2 | 3.74E-2 | 1.97E-2 | 1.37E-1 | 3.13E-2 | 1.86E-1 | 3.57E-3 | 3.95E-1 | 7.94E-3 | 2.33E-2 |
| | TDM | 2.20E-2 | 6.10E-2 | 2.29E-2 | 1.27E-1 | 1.02E-2 | 2.07E-2 | 2.02E-2 | 3.68E-2 | 2.99E-2 | 1.34E-1 | 2.51E-2 | 1.59E-1 | 1.03E-2 | 2.80E-1 | 5.78E-3 | 1.86E-2 |
| | ReMasker | 2.99E-2 | 1.14E-1 | 3.00E-2 | 1.38E-1 | 8.40E-2 | 1.71E-1 | 2.04E-2 | 2.98E-2 | 3.94E-2 | 2.77E-1 | 3.44E-2 | 2.01E-1 | 6.90E-2 | 9.91E-1 | 1.76E-2 | 2.88E-2 |
| | **NewImp** | 2.04E-2 | 1.70E-2 | 4.70E-3 | 1.96E-2 | 1.38E-2 | 1.78E-2 | 1.45E-2 | 2.77E-2 | 3.86E-3 | 8.63E-3 | 7.47E-3 | 8.12E-2 | 2.18E-2 | 6.56E-1 | 4.31E-3 | 1.59E-2 |

Table G.5: Performance of MAE and WASS metrics at 40% missing rate.

| Scenario | Model | BT MAE | BT WASS | BCD MAE | BCD WASS | CC MAE | CC WASS | CBV MAE | CBV WASS | IS MAE | IS WASS | PK MAE | PK WASS | QB MAE | QB WASS | WQW MAE | WQW WASS |
|---|---|---|---|---|---|---|---|---|---|---|---|---|---|---|---|---|---|
| MAR | CSDI_T | 0.92* | 2.80* | 0.89* | 17.47* | 0.84* | 2.75* | 0.82* | 3.82* | 0.70* | 16.04* | 0.89* | 13.29* | 0.62* | 20.06* | 0.77* | 4.05* |
| | MissDiff | 0.83* | 1.86* | 0.87* | 15.08* | 0.84* | 1.37* | 0.83* | 4.06* | 0.70* | 13.50* | 0.86* | 18.23* | 0.61* | 23.33* | 0.75* | 6.41* |
| | GAIN | 0.86* | 0.87 | 0.56* | 2.28* | 0.78* | 0.76* | 0.74* | 0.94* | 0.52* | 2.14* | 0.57 | 2.46 | 0.66* | 6.41* | 0.72* | 1.17* |
| | MIRACLE | 0.66* | 0.43* | 0.72* | 7.16 | 0.47 | **0.32** | 0.57* | 0.58* | 4.02* | 61.45* | 3.48* | 32.16* | 0.49* | 4.33 | 0.53 | 0.69 |
| | MIWAE | 0.69 | 0.59 | 0.53* | 2.10* | 0.78* | 0.74* | 0.83* | 1.10* | 0.53* | 2.58* | 0.58* | 2.11 | 0.52* | 5.79* | 0.61* | 0.85* |
| | Sink | 0.87* | 0.94 | 0.89* | 4.76* | 0.85* | 0.86* | 0.84* | 1.17* | 0.74* | 3.13* | 0.88* | 3.97* | 0.63* | 5.60* | 0.76* | 1.25* |
| | TDM | 0.84* | 0.93 | 0.82* | 4.31* | 0.77* | 0.75* | 0.78* | 1.05* | 0.64* | 2.57* | 0.81* | 3.61* | 0.58* | 5.37* | 0.73* | 1.19* |
| | ReMasker | **0.41*** | **0.22*** | 0.43 | 1.40 | 0.93* | 1.05* | 0.51* | 0.47* | 0.51* | 2.23* | 0.58* | 1.96 | 0.53 | **3.89** | 0.52* | **0.66*** |
| | **NewImp** | 0.71 | 0.79 | **0.38** | **1.32** | **0.43** | 0.35 | **0.40** | **0.37** | **0.40** | **1.67** | **0.46** | **1.68** | **0.43** | 4.08 | **0.50** | 0.70 |
| MCAR | CSDI_T | 0.71* | 1.76* | 0.73* | 15.43* | 0.85* | 2.77* | 0.82* | 3.91* | 0.76* | 14.66* | 0.72* | 11.68* | 0.57* | 19.81* | 0.78* | 4.15* |
| | MissDiff | 0.71* | 1.22* | 0.73* | 13.89* | 0.84* | 1.29* | 0.81* | 3.00* | 0.75* | 13.14* | 0.72* | 12.65* | 0.56* | 19.50* | 0.76* | 3.89* |
| | GAIN | 0.74* | 0.49* | 0.46* | 2.61* | 0.79* | 0.90* | 0.79* | 1.45* | 0.74* | 7.30* | 0.52* | 2.49 | 0.63* | 11.12 | 0.77* | 1.52* |
| | MIRACLE | 0.58 | 0.25* | 0.49* | 2.92* | 0.60* | **0.48*** | 0.66* | 0.99* | 2.74* | 59.66* | 3.20* | 51.98* | 0.40* | **4.28*** | 0.61* | 0.97* |
| | MIWAE | 0.62* | 0.32 | 0.51* | 3.50* | 0.79* | 0.84* | 0.84* | 1.53* | 0.66* | 6.68* | 0.53* | 2.67* | 0.50* | 7.86* | 0.68* | 1.17* |
| | Sink | 0.74* | 0.57* | 0.75* | 5.78* | 0.84* | 1.04* | 0.82* | 1.60* | 0.76* | 6.59* | 0.74* | 4.33* | 0.61* | 7.62* | 0.76* | 1.59* |
| | TDM | 0.70* | 0.52* | 0.65* | 4.90* | 0.79* | 0.93* | 0.78* | 1.48* | 0.67* | 5.71* | 0.65* | 3.74* | 0.53* | 7.06* | 0.74* | 1.50* |
| | ReMasker | **0.52** | **0.17*** | 0.37* | 1.97* | 1.02* | 1.54* | 0.64* | 0.88* | 0.55* | 5.64* | 0.47* | 2.13* | 0.44* | 4.88 | 0.59 | **0.97*** |
| | **NewImp** | 0.56 | 0.30 | **0.28** | **1.29** | **0.55** | 0.52 | **0.54** | **0.74** | **0.45** | **4.00** | **0.35** | **1.55** | **0.37** | 4.98 | **0.59** | 1.05 |
| MNAR | CSDI_T | 0.79* | 2.16* | 0.81* | 16.41* | 0.86* | 2.81* | 0.83* | 3.89* | 0.75* | 14.67* | 0.81* | 11.79* | 0.60* | 19.51* | 0.79* | 4.17* |
| | MissDiff | 0.75* | 1.39* | 0.79* | 14.93* | 0.84* | 1.23* | 0.83* | 3.35* | 0.74* | 13.45* | 0.78* | 14.07* | 0.60* | 20.56* | 0.76* | 4.29* |
| | GAIN | 0.76* | 0.62 | 0.85* | 7.92* | 0.86* | 1.11 | 0.96* | 2.16* | 1.15* | 13.95* | 0.84* | 6.52* | 1.20* | 31.45* | 1.01* | 2.49* |
| | MIRACLE | 0.68* | 0.41 | 0.77* | 12.61 | 0.62* | **0.53** | 0.68* | 1.05* | 2.79* | 58.80* | 3.24* | 51.35* | 0.63* | 12.68 | 0.61 | 1.03* |
| | MIWAE | 0.67 | 0.39 | 0.58* | 4.47* | 0.79* | 0.88* | 0.84* | 1.57* | 0.64* | 6.17* | 0.59* | 3.72* | 0.54* | 9.12* | 0.69* | 1.21* |
| | Sink | 0.77* | 0.67* | 0.82* | 7.39* | 0.85* | 1.09* | 0.83* | 1.65* | 0.76* | 6.46* | 0.81* | 5.87* | 0.63* | 8.78* | 0.77* | 1.64* |
| | TDM | 0.74* | 0.63* | 0.74* | 6.60* | 0.80* | 1.00* | 0.79* | 1.54* | 0.67* | 5.55* | 0.74* | 5.33* | 0.57* | 8.31* | 0.75* | 1.56* |
| | ReMasker | **0.59** | **0.32** | 0.46* | 3.22* | 1.02* | 1.63* | 0.66* | **0.92*** | 0.56* | 5.53* | 0.61* | 4.04* | 0.47 | **6.89** | 0.59* | 0.98 |
| | **NewImp** | 0.64 | 0.46 | **0.36** | **2.34** | **0.58** | 0.59 | **0.56** | 0.83 | **0.44** | **3.92** | **0.43** | **2.31** | **0.44** | 7.16 | **0.60** | 1.10 |

*Kindly Note*: The best results are **bolded** and the second best results are underliend. "*" marks the results that NewImp significantly outperform with $p$-value$< 0.05$ over paired samples $t$-test.

Table G.6: Standard deviation of MAE and WASS metrics at 40% missing rate.

| Scenario | Model | BT MAE | BT WASS | BCD MAE | BCD WASS | CC MAE | CC WASS | CBV MAE | CBV WASS | IS MAE | IS WASS | PK MAE | PK WASS | QB MAE | QB WASS | WQW MAE | WQW WASS |
|---|---|---|---|---|---|---|---|---|---|---|---|---|---|---|---|---|---|
| MAR | CSDI_T | 5.62E-2 | 4.89E-1 | 4.20E-2 | 2.52E0 | 4.95E-2 | 1.76E-1 | 1.12E-2 | 1.08E-1 | 1.79E-2 | 7.93E-1 | 1.60E-2 | 2.80E0 | 7.74E-3 | 4.41E-1 | 1.70E-2 | 1.24E-2 |
| | MissDiff | 2.52E-2 | 5.98E-1 | 2.08E-2 | 2.15E0 | 2.61E-2 | 1.82E-1 | 1.32E-2 | 1.23E0 | 2.35E-2 | 7.88E-1 | 4.90E-2 | 4.20E0 | 1.07E-2 | 4.73E0 | 1.73E-2 | 1.24E0 |
| | GAIN | 9.55E-2 | 1.65E-1 | 4.16E-2 | 2.92E-1 | 4.52E-2 | 1.07E-1 | 3.97E-2 | 1.08E-1 | 4.52E-2 | 2.47E-1 | 6.32E-2 | 7.18E-1 | 6.09E-2 | 9.92E-1 | 5.57E-2 | 1.49E-1 |
| | MIRACLE | 5.76E-2 | 8.09E-2 | 8.59E-2 | 6.27E0 | 1.86E-2 | 3.13E-2 | 1.89E-2 | 3.62E-2 | 6.15E-1 | 1.76E1 | 1.93E-1 | 3.20E0 | 3.28E-2 | 1.30E0 | 2.53E-2 | 4.69E-2 |
| | MIWAE | 6.81E-3 | 2.48E-2 | 3.73E-2 | 2.38E-1 | 1.74E-2 | 4.58E-2 | 1.57E-2 | 3.97E-2 | 2.50E-2 | 1.64E-1 | 3.00E-2 | 1.35E-1 | 1.48E-2 | 4.72E-1 | 1.52E-2 | 5.58E-3 |
| | Sink | 2.45E-2 | 5.19E-2 | 2.01E-2 | 1.44E-1 | 2.64E-2 | 3.44E-2 | 9.54E-3 | 3.82E-2 | 3.42E-2 | 1.43E-1 | 2.53E-2 | 1.01E-1 | 5.68E-3 | 2.33E-1 | 1.93E-2 | 4.80E-2 |
| | TDM | 2.33E-2 | 6.30E-2 | 1.75E-2 | 1.26E-1 | 2.12E-2 | 2.78E-2 | 6.42E-3 | 2.93E-2 | 3.05E-2 | 1.31E-1 | 2.60E-2 | 1.36E-1 | 1.06E-2 | 1.98E-1 | 1.47E-2 | 3.44E-2 |
| | ReMasker | 1.17E-2 | 1.53E-3 | 1.11E-2 | 7.93E-2 | 4.22E-2 | 1.51E-1 | 1.44E-2 | 3.52E-2 | 3.12E-2 | 1.28E-1 | 4.30E-2 | 2.35E-1 | 1.68E-2 | 3.13E-1 | 1.84E-2 | 1.91E-2 |
| | **NewImp** | 3.84E-2 | 9.15E-2 | 2.98E-2 | 1.76E-1 | 1.54E-2 | 1.61E-2 | 2.06E-2 | 4.60E-2 | 9.12E-3 | 8.56E-2 | 2.65E-2 | 2.37E-1 | 1.68E-2 | 2.47E-1 | 7.86E-3 | 1.84E-2 |
| MCAR | CSDI_T | 1.48E-2 | 1.26E-1 | 7.13E-3 | 5.01E-1 | 1.65E-2 | 9.81E-2 | 3.60E-3 | 4.53E-2 | 7.26E-3 | 1.17E-1 | 8.37E-3 | 9.05E-1 | 4.95E-3 | 1.15E-1 | 4.98E-3 | 2.86E-2 |
| | MissDiff | 2.06E-2 | 4.36E-1 | 4.25E-3 | 2.93E-1 | 9.19E-3 | 1.63E-1 | 3.52E-3 | 6.06E-1 | 6.67E-3 | 5.26E-2 | 8.46E-3 | 4.44E-1 | 7.08E-3 | 2.29E0 | 3.42E-3 | 4.09E-1 |
| | GAIN | 5.60E-2 | 5.41E-2 | 4.64E-2 | 5.45E-1 | 1.66E-2 | 4.70E-2 | 3.97E-2 | 1.68E-1 | 7.12E-2 | 1.29E0 | 3.50E-2 | 3.89E-1 | 1.05E-1 | 4.04E0 | 2.79E-2 | 9.53E-2 |
| | MIRACLE | 3.75E-2 | 1.74E-2 | 1.64E-2 | 2.28E-1 | 8.23E-3 | 1.64E-2 | 8.89E-3 | 1.23E-2 | 3.45E-2 | 2.51E0 | 7.71E-2 | 2.75E0 | 8.06E-3 | 2.20E-1 | 2.45E-3 | 9.21E-3 |
| | MIWAE | 1.15E-2 | 3.09E-2 | 1.20E-2 | 1.68E-1 | 1.19E-2 | 1.74E-2 | 2.14E-2 | 7.26E-2 | 2.21E-2 | 2.49E-1 | 1.79E-3 | 1.13E-1 | 7.84E-3 | 1.36E-1 | 2.35E-3 | 7.99E-3 |
| | Sink | 1.77E-2 | 5.29E-2 | 1.02E-3 | 1.47E-1 | 7.96E-3 | 1.07E-2 | 6.43E-3 | 2.29E-2 | 6.07E-3 | 8.95E-2 | 6.39E-3 | 3.77E-2 | 2.21E-3 | 2.36E-1 | 2.72E-3 | 1.78E-2 |
| | TDM | 1.21E-2 | 5.55E-2 | 4.32E-3 | 1.34E-1 | 7.35E-3 | 6.43E-3 | 7.29E-3 | 2.60E-2 | 6.14E-3 | 4.51E-2 | 1.00E-2 | 6.14E-2 | 4.32E-3 | 1.81E-1 | 2.10E-3 | 5.86E-3 |
| | ReMasker | 2.34E-2 | 2.34E-2 | 8.99E-3 | 1.32E-1 | 7.16E-2 | 2.70E-1 | 7.18E-3 | 2.55E-2 | 2.23E-2 | 2.73E-1 | 1.56E-2 | 1.60E-1 | 9.66E-3 | 3.15E-1 | 4.74E-3 | 1.55E-2 |
| | **NewImp** | 1.10E-2 | 1.59E-2 | 2.77E-3 | 6.27E-2 | 1.22E-2 | 2.43E-2 | 9.57E-3 | 2.58E-2 | 5.96E-3 | 1.52E-1 | 6.26E-3 | 8.57E-2 | 6.42E-3 | 2.24E-1 | 3.51E-3 | 1.21E-2 |
| MNAR | CSDI_T | 2.46E-2 | 1.20E-1 | 1.89E-2 | 1.73E0 | 1.64E-2 | 9.76E-2 | 1.00E-2 | 1.12E-1 | 7.74E-3 | 1.29E-1 | 2.37E-2 | 1.38E0 | 9.74E-3 | 2.76E-1 | 6.99E-3 | 2.32E-2 |
| | MissDiff | 3.18E-2 | 3.53E-1 | 5.71E-3 | 1.11E0 | 6.41E-3 | 5.92E-2 | 6.35E-3 | 5.98E-1 | 1.44E-2 | 9.89E-2 | 1.14E-2 | 2.02E0 | 4.89E-3 | 2.33E0 | 3.86E-3 | 7.19E-1 |
| | GAIN | 3.59E-2 | 2.80E-2 | 6.99E-2 | 9.28E-1 | 1.27E-1 | 3.41E-1 | 1.27E-1 | 5.75E-1 | 7.48E-2 | 1.94E0 | 6.59E-2 | 1.10E0 | 5.06E-2 | 3.08E0 | 1.24E-1 | 5.38E-1 |
| | MIRACLE | 2.73E-2 | 6.77E-2 | 1.06E-1 | 9.92E0 | 7.74E-3 | 1.80E-2 | 4.77E-3 | 1.52E-2 | 6.96E-2 | 2.75E0 | 3.64E-2 | 2.04E0 | 1.35E-2 | 3.87E0 | 5.02E-3 | 1.09E-2 |
| | MIWAE | 1.68E-2 | 2.03E-2 | 2.21E-2 | 3.33E-1 | 1.42E-2 | 5.43E-2 | 1.51E-2 | 6.14E-2 | 6.26E-3 | 1.54E-1 | 3.69E-2 | 4.10E-1 | 1.35E-2 | 5.96E-1 | 5.12E-3 | 2.25E-2 |
| | Sink | 5.21E-3 | 5.52E-2 | 7.91E-3 | 2.08E-1 | 7.14E-3 | 4.09E-2 | 1.51E-3 | 1.36E-2 | 6.10E-3 | 3.67E-2 | 1.25E-2 | 1.79E-1 | 6.44E-3 | 3.55E-1 | 1.68E-3 | 2.31E-2 |
| | TDM | 1.12E-2 | 6.53E-2 | 1.15E-2 | 2.41E-1 | 8.22E-3 | 4.59E-2 | 2.05E-3 | 2.00E-2 | 2.66E-2 | 2.49E-1 | 1.84E-2 | 2.26E-1 | 5.12E-3 | 4.67E-1 | 4.06E-3 | 3.10E-2 |
| | ReMasker | 7.23E-3 | 4.58E-2 | 2.96E-2 | 3.39E-1 | 6.75E-2 | 2.22E-1 | 7.57E-3 | 2.93E-2 | 1.43E-2 | 9.16E-2 | 2.25E-2 | 3.43E-1 | 1.61E-2 | 5.20E-1 | 1.10E-3 | 2.91E-2 |
| | **NewImp** | 2.87E-2 | 1.07E-1 | 6.75E-3 | 1.35E-1 | 1.62E-2 | 4.75E-2 | 4.48E-3 | 2.24E-2 | 6.74E-3 | 9.11E-2 | 5.62E-3 | 5.59E-2 | 3.52E-3 | 1.61E-1 | 8.02E-3 | 2.34E-2 |

Table G.7: Performance of MAE and WASS metrics at 50% missing rate.

| Scenario | Model | BT MAE | BT WASS | BCD MAE | BCD WASS | CC MAE | CC WASS | CBV MAE | CBV WASS | IS MAE | IS WASS | PK MAE | PK WASS | QB MAE | QB WASS | WQW MAE | WQW WASS |
|---|---|---|---|---|---|---|---|---|---|---|---|---|---|---|---|---|---|
| MAR | CSDI_T | 0.85* | 2.61* | 0.84* | 17.30* | 0.84* | 2.82* | 0.81* | 3.79* | 0.72* | 15.46* | 0.84* | 13.20* | 0.60* | 19.79* | 0.78* | 4.09* |
| | MissDiff | 0.78 | 1.97* | 0.82* | 15.22* | 0.84* | 1.34* | 0.82* | 3.35* | 0.72* | 13.67* | 0.80* | 15.05* | 0.59* | 24.55* | 0.77* | 5.95* |
| | GAIN | 0.74 | 0.69 | 0.55* | 2.80 | 0.77* | 0.85* | 0.74* | 1.05* | 0.60* | 3.06* | 0.52 | 2.22 | 0.67* | 8.36* | 0.88* | 1.84* |
| | MIRACLE | 0.66 | 0.49 | 0.85* | 7.33* | 0.52 | **0.41*** | 0.63* | 0.78* | 3.21* | 50.79* | 3.88* | 48.31* | 0.52* | 5.27 | 0.58 | 0.87* |
| | MIWAE | 0.63 | 0.50 | 0.51* | 2.51 | 0.78* | 0.79* | 0.84* | 1.23* | 0.56* | 3.28* | 0.55* | 2.26 | 0.52* | 6.70* | 0.66* | 1.05 |
| | Sink | 0.82* | 0.83 | 0.84* | 5.38* | 0.86* | 1.00* | 0.83* | 1.30* | 0.75* | 3.93* | 0.83* | 4.40* | 0.61* | 6.44* | 0.78* | 1.47* |
| | TDM | 0.78 | 0.81 | 0.79* | 5.05* | 0.80* | 0.90* | 0.78* | 1.19* | 0.68* | 3.43* | 0.77* | 4.03* | 0.56* | 6.13* | 0.76* | 1.42* |
| | ReMasker | **0.40** | **0.28** | **0.39*** | **1.47*** | 0.98* | 1.35* | 0.51* | **0.54** | 0.52* | 2.92* | 0.51 | 2.01 | 0.53 | 4.70 | 0.57 | **0.85*** |
| | **NewImp** | 0.67 | 0.67 | 0.44 | 2.06 | **0.51** | **0.48** | **0.47** | 0.55 | **0.45** | **2.20** | 0.48 | 2.17 | **0.42** | 4.72 | 0.60 | 1.00 |
| MCAR | CSDI_T | 0.73* | 1.86* | 0.74* | 15.79* | 0.85* | 2.71* | 0.82* | 3.99* | 0.75* | 14.56* | 0.72* | 11.31* | 0.57* | 19.45* | 0.79* | 4.31* |
| | MissDiff | 0.72* | 1.44* | 0.74* | 14.15* | 0.85* | 1.46* | 0.81* | 2.83* | 0.75* | 13.14* | 0.71* | 11.70* | 0.56* | 20.15* | 0.76* | 3.43* |
| | GAIN | 0.70* | 0.52* | 0.50* | 3.74* | 0.87* | 1.29* | 1.11 | 3.32 | 0.92* | 12.89* | 0.65* | 4.72* | 1.03* | 31.73* | 0.86 | 2.22 |
| | MIRACLE | 0.63 | 0.34 | 0.63* | 6.17* | 0.66 | **0.68*** | 0.74* | 1.35* | 2.46* | 60.64* | 2.97* | 57.33* | 0.55* | 8.14* | 0.65* | 1.21* |
| | MIWAE | 0.64* | 0.38 | 0.55* | 4.71* | 0.81* | 1.03* | 0.84* | 1.77* | 0.68* | 8.31* | 0.56* | 3.63* | 0.52* | 9.83* | 0.70* | 1.39 |
| | Sink | 0.74* | 0.68* | 0.75* | 7.30* | 0.84* | 1.28* | 0.82* | 1.91* | 0.75* | 8.27* | 0.74* | 5.64* | 0.61* | 9.24* | 0.76* | 1.92* |
| | TDM | 0.72* | 0.64* | 0.67* | 6.36* | 0.81* | 1.18* | 0.79* | 1.80* | 0.69* | 7.53* | 0.68* | 5.05* | 0.55* | 8.98* | 0.75* | 1.83* |
| | ReMasker | **0.58** | **0.23** | 0.44* | 3.17* | 1.03* | 1.96* | 0.75* | 1.22 | 0.63* | 8.24* | 0.51* | 2.81* | 0.49* | 7.32 | 0.63 | **1.16*** |
| | **NewImp** | 0.59 | 0.38 | **0.34** | 2.16 | **0.65** | 0.84 | **0.64** | **1.18** | **0.48** | **5.12** | **0.40** | **2.22** | **0.40** | **6.92** | 0.64 | 1.40 |
| MNAR | CSDI_T | 0.75* | 2.07* | 0.78* | 15.16* | 0.86* | 2.79* | 0.82* | 3.95* | 0.75* | 14.54* | 0.80* | 12.42* | 0.60* | 19.50* | 0.79* | 4.23* |
| | MissDiff | 0.72 | 1.34 | 0.77* | 14.78* | 0.85* | 1.44* | 0.82* | 3.08* | 0.73* | 13.52* | 0.77* | 11.83* | 0.59* | 20.34* | 0.77* | 3.65* |
| | GAIN | 0.76 | 0.74 | 0.99* | 12.08* | 0.93* | 1.64* | 1.22* | 3.97* | 1.67* | 31.57* | 0.92* | 9.02* | 1.37* | 49.74* | 1.20* | 4.30* |
| | MIRACLE | 0.83* | 0.74 | 1.14* | 37.98 | 0.70* | **0.79*** | 0.74* | 1.39* | 2.52* | 61.17* | 3.14* | 62.26* | 0.71* | 14.79* | 0.66* | 1.31* |
| | MIWAE | 0.73 | 0.63* | 0.61* | 5.73* | 0.82* | 1.08 | 0.85* | 1.84* | 0.66* | 7.85* | 0.61* | 4.33* | 0.53* | 10.68* | 0.71* | 1.47 |
| | Sink | 0.78* | 0.86* | 0.79* | 8.47* | 0.86* | 1.35* | 0.83* | 1.98* | 0.76* | 8.19* | 0.78* | 6.54* | 0.63* | 10.57* | 0.77* | 2.01* |
| | TDM | 0.76* | 0.83 | 0.73* | 7.72* | 0.83* | 1.26* | 0.80* | 1.87* | 0.69* | 7.30* | 0.72* | 6.04* | 0.57* | 10.20* | 0.75* | 1.91* |
| | ReMasker | **0.61*** | **0.42*** | 0.55* | 5.22* | 0.99* | 1.86* | 0.76* | 1.29 | 0.76* | 9.82* | 0.63* | 5.39* | 0.50* | 9.25* | 0.65 | **1.22*** |
| | **NewImp** | 0.74 | 0.84 | **0.39** | 3.12 | **0.69** | 0.98 | **0.65** | **1.23** | **0.47** | **5.03** | **0.48** | **3.31** | **0.44** | **8.38** | 0.65 | 1.47 |

*Kindly Note*: The best results are **bolded** and the second best results are underliend. "*" marks the results that NewImp significantly outperform with $p$-value$< 0.05$ over paired samples $t$-test.

Table G.8: Standard deviation of MAE and WASS metrics at 50% missing rate.

| Scenario | Model | BT MAE | BT WASS | BCD MAE | BCD WASS | CC MAE | CC WASS | CBV MAE | CBV WASS | IS MAE | IS WASS | PK MAE | PK WASS | QB MAE | QB WASS | WQW MAE | WQW WASS |
|---|---|---|---|---|---|---|---|---|---|---|---|---|---|---|---|---|---|
| MAR | CSDI_T | 8.31E-2 | 4.42E-1 | 2.83E-2 | 2.31E0 | 3.66E-2 | 1.29E-1 | 1.34E-2 | 5.51E-2 | 2.39E-2 | 9.07E-1 | 3.55E-2 | 2.14E0 | 2.05E-2 | 5.84E-1 | 3.11E-2 | 2.52E-2 |
| | MissDiff | 4.50E-2 | 5.54E-1 | 5.12E-3 | 1.98E0 | 1.24E-2 | 1.09E-1 | 8.44E-3 | 8.50E-1 | 3.78E-2 | 4.65E-1 | 2.17E-2 | 2.02E0 | 2.51E-2 | 3.72E0 | 2.49E-2 | 1.10E0 |
| | GAIN | 5.10E-2 | 1.42E-1 | 6.41E-2 | 5.67E-1 | 4.97E-2 | 9.93E-2 | 3.91E-2 | 9.48E-2 | 2.46E-2 | 2.48E-1 | 7.03E-2 | 8.68E-1 | 8.40E-2 | 1.77E0 | 1.01E-1 | 3.67E-1 |
| | MIRACLE | 1.50E-1 | 2.05E-1 | 9.31E-2 | 1.94E0 | 3.74E-2 | 6.26E-2 | 1.45E-2 | 2.64E-2 | 5.95E-1 | 1.68E1 | 5.56E-1 | 1.26E1 | 4.19E-2 | 6.07E-1 | 2.96E-2 | 5.79E-2 |
| | MIWAE | 6.94E-2 | 1.35E-1 | 1.75E-2 | 1.16E-1 | 2.18E-2 | 4.43E-2 | 1.89E-2 | 1.02E-1 | 3.21E-2 | 2.02E-1 | 3.10E-2 | 2.21E-1 | 4.53E-2 | 3.92E-1 | 2.57E-2 | 5.02E-2 |
| | Sink | 4.60E-2 | 1.38E-1 | 8.04E-3 | 8.61E-2 | 2.40E-2 | 3.24E-2 | 1.17E-2 | 6.38E-2 | 3.09E-2 | 1.35E-1 | 1.16E-2 | 4.24E-2 | 2.72E-2 | 7.38E-2 | 3.42E-2 | 8.69E-2 |
| | TDM | 5.43E-2 | 1.61E-1 | 1.16E-2 | 7.27E-2 | 2.37E-2 | 4.04E-2 | 1.32E-2 | 7.26E-2 | 1.92E-2 | 9.41E-2 | 1.34E-2 | 9.02E-2 | 2.84E-2 | 6.32E-2 | 3.02E-2 | 7.58E-2 |
| | ReMasker | 7.39E-2 | 1.44E-1 | 2.99E-2 | 1.41E-1 | 9.30E-2 | 2.22E-1 | 2.35E-2 | 3.19E-2 | 2.98E-2 | 2.94E-1 | 7.54E-2 | 6.08E-1 | 6.79E-2 | 7.58E-1 | 2.26E-2 | 4.20E-2 |
| | **NewImp** | 1.07E-1 | 2.34E-1 | 3.02E-2 | 2.10E-1 | 2.83E-2 | 5.71E-2 | 1.19E-2 | 2.15E-2 | 1.51E-2 | 1.69E-1 | 1.42E-2 | 8.97E-2 | 1.69E-2 | 1.36E-1 | 4.13E-2 | 9.67E-2 |
| MCAR | CSDI_T | 1.28E-2 | 1.58E-1 | 5.14E-3 | 1.23E0 | 8.83E-3 | 9.46E-2 | 2.93E-3 | 3.67E-2 | 7.87E-3 | 9.01E-2 | 6.38E-3 | 6.10E-1 | 5.04E-3 | 1.69E-1 | 1.11E-2 | 7.61E-2 |
| | MissDiff | 2.62E-3 | 3.37E-1 | 1.01E-2 | 3.62E-1 | 4.33E-3 | 1.06E-1 | 3.15E-3 | 3.18E-1 | 9.26E-3 | 9.37E-2 | 4.25E-3 | 5.98E-1 | 4.81E-3 | 1.26E0 | 1.71E-3 | 3.02E-1 |
| | GAIN | 1.71E-2 | 3.47E-2 | 2.42E-2 | 2.75E-1 | 4.53E-2 | 1.44E-1 | 2.62E-1 | 1.40E0 | 1.09E-1 | 2.43E0 | 2.01E-2 | 3.35E-1 | 1.38E-1 | 7.19E0 | 1.41E-1 | 6.71E-1 |
| | MIRACLE | 2.19E-2 | 2.36E-2 | 7.61E-2 | 1.93E0 | 1.32E-2 | 2.67E-2 | 1.55E-2 | 3.61E-2 | 3.58E-2 | 2.18E0 | 4.86E-2 | 2.18E0 | 2.15E-2 | 1.96E-1 | 2.25E-3 | 5.85E-3 |
| | MIWAE | 6.77E-3 | 4.45E-2 | 2.86E-2 | 3.40E-1 | 1.15E-2 | 3.66E-2 | 9.27E-3 | 4.98E-2 | 1.19E-2 | 1.97E-1 | 1.72E-2 | 3.34E-1 | 1.09E-2 | 3.17E-1 | 1.98E-3 | 1.68E-2 |
| | Sink | 9.45E-3 | 2.19E-2 | 5.14E-3 | 7.93E-2 | 4.00E-3 | 1.70E-2 | 2.62E-3 | 2.17E-2 | 1.92E-3 | 5.94E-2 | 3.52E-3 | 1.46E-1 | 4.69E-3 | 7.05E-2 | 1.96E-3 | 2.86E-2 |
| | TDM | 7.85E-3 | 2.24E-2 | 7.53E-3 | 9.35E-2 | 7.82E-3 | 2.75E-2 | 4.80E-3 | 2.12E-2 | 3.76E-3 | 1.02E-1 | 9.26E-3 | 1.67E-1 | 3.48E-3 | 1.87E-1 | 2.66E-3 | 5.92E-3 |
| | ReMasker | 5.17E-2 | 6.46E-2 | 6.65E-4 | 4.77E-2 | 3.95E-2 | 1.45E-1 | 2.96E-3 | 1.48E-2 | 1.13E-2 | 1.97E-1 | 1.18E-2 | 2.05E-1 | 1.34E-2 | 4.30E-1 | 1.06E-3 | 2.08E-2 |
| | **NewImp** | 1.22E-2 | 2.48E-2 | 3.72E-3 | 1.03E-1 | 9.87E-3 | 3.21E-2 | 5.17E-3 | 3.13E-2 | 8.34E-3 | 1.56E-1 | 7.22E-3 | 6.63E-2 | 6.53E-3 | 2.43E-1 | 1.57E-3 | 9.71E-3 |
| MNAR | CSDI_T | 5.67E-3 | 1.31E-1 | 1.04E-2 | 7.15E-1 | 2.05E-2 | 9.53E-2 | 1.23E-2 | 1.30E-1 | 6.97E-3 | 1.06E-1 | 1.36E-2 | 9.70E-1 | 4.20E-3 | 2.68E-1 | 6.23E-3 | 5.50E-2 |
| | MissDiff | 7.33E-3 | 3.18E-1 | 6.20E-3 | 7.94E-1 | 1.48E-2 | 1.30E-1 | 6.44E-3 | 3.30E-1 | 9.96E-3 | 1.17E-1 | 1.01E-2 | 8.14E-1 | 8.28E-3 | 1.97E0 | 4.54E-3 | 1.73E-1 |
| | GAIN | 3.37E-2 | 8.08E-2 | 3.52E-2 | 8.61E-1 | 7.13E-2 | 2.53E-1 | 8.83E-2 | 5.11E-1 | 1.55E-1 | 4.35E0 | 4.63E-2 | 1.21E0 | 1.62E-1 | 1.30E1 | 7.71E-2 | 5.63E-1 |
| | MIRACLE | 4.62E-2 | 1.54E-1 | 1.55E-1 | 2.46E1 | 1.56E-2 | 4.66E-2 | 1.35E-2 | 4.39E-2 | 2.21E-2 | 1.78E0 | 9.72E-2 | 3.67E0 | 3.60E-2 | 1.71E0 | 1.24E-2 | 3.05E-2 |
| | MIWAE | 2.88E-2 | 7.04E-2 | 3.13E-2 | 2.81E-1 | 6.52E-3 | 5.64E-2 | 1.06E-2 | 5.53E-2 | 2.19E-2 | 3.10E-1 | 1.69E-2 | 2.92E-1 | 1.75E-2 | 3.51E-1 | 4.18E-3 | 3.20E-2 |
| | Sink | 4.90E-3 | 1.42E-2 | 6.82E-3 | 2.35E-1 | 7.10E-3 | 3.41E-2 | 4.21E-3 | 3.28E-2 | 6.06E-4 | 9.24E-2 | 1.26E-2 | 3.20E-1 | 6.81E-3 | 1.42E-1 | 6.82E-3 | 5.11E-2 |
| | TDM | 2.87E-3 | 1.46E-2 | 1.06E-2 | 2.95E-1 | 7.99E-3 | 3.36E-2 | 5.73E-3 | 3.48E-2 | 1.56E-2 | 1.77E-1 | 1.51E-2 | 3.73E-1 | 6.81E-3 | 1.42E-1 | 6.82E-3 | 5.11E-2 |
| | ReMasker | 2.52E-2 | 4.11E-2 | 2.60E-2 | 4.54E-1 | 5.75E-2 | 2.40E-1 | 1.06E-2 | 4.72E-2 | 3.77E-2 | 4.46E-1 | 3.96E-2 | 5.44E-1 | 1.13E-2 | 1.19E-1 | 4.54E-3 | 1.92E-2 |
| | **NewImp** | 4.81E-3 | 1.42E-2 | 8.08E-3 | 1.42E-1 | 1.86E-2 | 8.53E-2 | 1.61E-2 | 7.67E-2 | 2.97E-3 | 8.78E-2 | 1.42E-2 | 1.95E-1 | 6.32E-3 | 2.48E-1 | 1.11E-2 | 4.15E-2 |

